# An effective framework for estimating individualized treatment rules

**Joowon Lee**[1], **Jared D. Huling**[2], **Guanhua Chen**[1]
[1] University of Wisconsin-Madison, [2] University of Minnesota
joowon.lee@wisc.edu, huling@umn.edu, gchen25@wisc.edu

## Abstract

Estimating individualized treatment rules (ITRs) is fundamental in causal inference, particularly for precision medicine applications. Traditional ITR estimation methods rely on inverse probability weighting (IPW) to address confounding factors and $L_1$-penalization for simplicity and interpretability. However, IPW can introduce statistical bias without precise propensity score modeling, while $L_1$-penalization makes the objective non-smooth, leading to computational bias and requiring subgradient methods. In this paper, we propose a unified ITR estimation framework formulated as a constrained, weighted, and smooth convex optimization problem. The optimal ITR can be robustly and effectively computed by projected gradient descent. Our comprehensive theoretical analysis reveals that weights that balance the spectrum of a 'weighted design matrix' improve both the optimization and likelihood landscapes, yielding improved computational and statistical estimation guarantees. In particular, this is achieved by distributional covariate balancing weights, which are model-free alternatives to IPW. Extensive simulations and applications demonstrate that our framework achieves significant gains in both robustness and effectiveness for ITR learning against existing methods.

## 1 Introduction

Traditional medicine often uses a "one-size-fits-all" approach where the same treatment is applied to all patients regardless of their unique attributes, aiming to find a single optimal treatment that may be most effective for a broad population. However, since not everyone fits the mold, precision medicine has been popular in medical research, emphasizing personalized treatment based on a patient's unique characteristics [11].

One key element of precision medicine is the estimation of *individualized treatment regime* (ITR), also known as treatment policy or policy. ITRs consider various factors like demographics and socio-psychological aspects to optimize treatment decisions, thereby maximizing individual outcomes. Many frameworks have been proposed for learning ITRs [41, 54, 57, 7, 58, 60, 56] and the related *conditional average treatment effect* (CATE) estimation problem [3, 53, 46, 13, 36]. For model interpretation and other practical concerns, researchers often focus on pre-specified rule classes. For example, the method proposed in [2] focuses on estimating ITRs using shallow-depth decision trees.

The central task in the ITR estimation is controlling confounding factors to isolate the causal effect of treatment from other factors that may influence the outcome. A standard approach is by weighting each sample using their *inverse probability weights* (IPWs), which requires specifying a propensity score model. However, propensity score methods have long been known to be highly sensitive to model misspecification, which yields biased estimates of causal effects [25, 30]. *Distributional covariate balancing weights* (DCBWs) [24, 27] are modern alternatives to IPWs, directly minimizing the distance between empirical covariate distributions induced by the weights.

ITR estimation can be framed as an optimization or *maximum likelihood estimation* (MLE) problem, where the choice of weights directly impacts both the optimization and likelihood landscapes. This, in

turn, affects the computational and statistical errors in the ITR estimation, as well as the performance of the optimization algorithm. The key contribution of this work is to bridge the gap between causal inference and optimization perspectives in ITR estimation. Our main finding is both simple and surprising: *Covariate balancing weights not only address confounding but also yield near-optimal optimization and likelihood landscapes for the ITR estimation.* We show this claim by formulating the ITR estimation problem as a constrained, weighted, and smooth convex optimization problem. Our formulation is grounded in *angle-based direct learning* (AD-Learning) framework [39], which is a recent framework for multi-category treatments with a linear decision function class assumption. While we focus on multi-category treatments—a relatively underexplored area—we believe that the principles are equally applicable to binary treatments with a linear decision function.

Besides advancing our theoretical understanding of the impact of covariate balancing weights, we propose a unified computational and statistical framework for the ITR estimation. Notably, we introduce a hard $L_1$-ball constraint to promote sparsity in regression coefficients, replacing the traditional soft $L_1$-penalization. This approach keeps the optimization objective smooth, enabling fast and accurate ITR estimation using *projected gradient descent* (PGD). Furthermore, we integrate techniques such as (1) variable screening, (2) outcome augmentation, and (3) inverse variance weighting to account for heteroscedastic errors, which together show a synergistic effect.

**Related Work.** Most existing ITR approaches use IPW, which requires specifying a propensity score model for confounding control. Recent developments in causal inference have also introduced robust approaches that directly estimate weights based on balance-seeking objectives rather than relying solely on propensity scores. These approaches include entropy balancing weights [20], stable balancing weights [61] as well as DCBWs [24, 27]. However, despite the weights serving a critical role in the estimation process, these advancements have mainly focused on improving *average treatment effect* (ATE) and are largely under-explored in the ITR literature. To the best of our knowledge, this is the first work to apply DCBWs for ITR-Learning under a weighted optimization framework. While recent works have taken a similar approach by utilizing weighting schemes and directly optimizing the weights rather than relying on estimated IPW [28, 26], our method differs in key aspects. Unlike [26], which involves the evaluation of policy effects, our approach directly learns the optimal decision function, bypassing the need for intermediate policy effect evaluations. Although [28] introduced the concept of retargeting the population covariate distribution, which is conceptually related to our analysis, their approach relies on a reference policy, whereas our method does not. We also explore various statistical techniques to improve ITR-Learning. First, we perform *variable screening* using the distance covariance test [48] to retain key effect modifiers and precision variables that directly impact the outcome. Second, [50, 9] introduced *outcome augmentation* to minimize estimator variance in clinical trials and observational studies by IPW involving binary treatments. Building on these foundations, our work extends these approaches to a broader range of weighting schemes, such as DCBW particularly for multi-category treatments. Lastly, with inverse variance weighting, as introduced in *stabilized angle-based direct learning* (SABD-Learning) to address heteroscedasticity [44], this combined strategy enhances estimator precision and robustness.

**Contributions.** We summarize our contributions below.

- **Unified Framework of ITR-Learning**: We introduce a novel framework that addresses the limitations of existing ITR estimation methods by formulating the problem as a constrained, weighted, and smooth convex optimization problem. Under the framework, we propose a PGD algorithm for ITR-Learning under a hard $L_1$-constraint.

- **Improved Computational and Statistical Guarantees**: We establish convergence guarantees (Theorem 3.3). Furthermore, under mild assumptions, we demonstrate that the parameter of ITR-Learning can be consistently estimated with high probability (Theorem 3.5) and provide computational and sample complexity. In both cases, we demonstrate that using covariate balancing weights controls confounding factors and leads to better optimization and likelihood landscapes, resulting in improved computational and statistical performance.

- **Statistical Framework of ITR-Learning**: Our main contribution lies in providing a unified framework that combines DCBWs, variable screening, outcome augmentation, and inverse variance weighting in a synergistic and effective way. We demonstrated theoretical justifications for the combined approach, which, to our knowledge, have not been previously established.

## 2 Methods for ITR Estimation

### 2.1 Model Setup

**Individualized Treatment Rule and Optimal Decision Function.** We observe a random vector $(\mathbf{X}, A, Y)$ where $\mathbf{X} = (X_1, X_2, \ldots, X_p) \in \mathcal{X}$ denotes the $p$-dimensional vector of pre-treatment covariates, a received treatment $A \in \mathcal{A} = \{1, 2, \ldots, K\}$, and the corresponding outcome $Y$. Without loss of generality, we assume higher $Y$ values are more favorable. Let $Y(a)$ denote the random variable that describes the potential outcome that would have been observed were the individual assigned to treatment $a \in \mathcal{A}$. We make the following standard assumptions using the potential outcome framework [43]: (i) Stable Unit Treatment Value Assumption (SUTVA): $Y = Y(A)$, (ii) positivity: $0 < \pi(a, \mathbf{X}) := \mathbb{P}(A = a|\mathbf{X}) < 1$ for all $a \in \mathcal{A}$ and all $\mathbf{X} \in \mathcal{X}$, (iii) no unmeasured confounding: $Y(a) \perp A|\mathbf{X}$ for any $a \in \mathcal{A}$.

An ITR, $d(\mathbf{x})$, is a function mapping each covariate $\mathbf{x}$ to one of the $K$ treatments. According to [41], the ITR $d$ can be measured by the value function $V(d)$:

$$V(d) := \mathbb{E}[Y(d(\mathbf{X}))] = \mathbb{E}[Y|A = d(\mathbf{X})] = \mathbb{E}\left[\frac{Y \ \mathbf{1}(A = d(\mathbf{X}))}{\pi(A, \mathbf{X})}\right]$$

Then the optimal ITR $d^{\text{opt}}$ is defined as a function that maximizes the expected potential outcome $d^{\text{opt}}(\mathbf{x}) \in \arg\max_{d \in \mathcal{D}} V(d)$ among all functions belonging to the treatment rule class ($\mathcal{D}$). Building on this, we adopt the following working model:

$$Y = \mu(\mathbf{X}) + \sum_{k=1}^{K} \delta_k(\mathbf{X})\mathbf{I}(A = k) + \epsilon, \tag{1}$$

where $\mu(\mathbf{X})$ is treatment-free effect, $\delta_k(\mathbf{X})$ is interaction effect between covariates and $k$th treatment, and $\epsilon$ is a random noise with mean zero and variance $\sigma^2(A, \mathbf{X})$. For model identifiability, we assume $\sum_{k=1}^{K} \delta_k(\mathbf{x}) = 0$ for all $\mathbf{x}$. Under this model, $\delta_k(\mathbf{X})$ determines the optimal ITR, while $\mu(\mathbf{x})$ has no impact on the ITR. One approach to identifying the optimal ITR is to use $K$ separate decision functions for each treatment with the sum-to-zero constraint, but this approach can be computationally inefficient [55].

Instead, one can opt for simplex coding as an alternative, which inherently satisfies the sum-to-zero constraint. In AD-Learning [39], each of the $K$ treatments is represented as a vertex simplex on $\mathbb{R}^{K-1}$, denoted as $\mathbf{u}_k$, $k = 1, \ldots, K$. In particular, $\mathbf{u}_k$ is defined as

$$\mathbf{u}_k = \begin{cases} \frac{1}{\sqrt{K-1}}\mathbf{1}_{K-1} & \text{if } k = 1, \\ -\frac{1+\sqrt{K}}{\sqrt{(K-1)^3}}\mathbf{1}_{K-1} + \sqrt{\frac{K}{K-1}}\mathbf{e}_{k-1} & \text{if } k \geq 2, \end{cases}$$

where $\mathbf{1}_{K-1}$ is a $K-1$ dimensional vector with entries 1, and $\mathbf{e}_{k-1}$ is a $K-1$ dimensional vector with entries 0 except its $k$-th entry being 1. This vertex simplex has $K$ vertices with equal angles between them and an origin at the center of the simplex. All $\mathbf{u}_k$ have the same Euclidean norm 1. Then the optimal ITR is reformulated as follows:

$$d^{\text{opt}}(\mathbf{x}) = \underset{k \in \{1,\ldots,K\}}{\arg\max} \ \mathbb{E}[Y|\mathbf{x}, A = k] = \underset{k \in \{1,\ldots,K\}}{\arg\max} \ \mathbf{u}_k^T \underbrace{\mathbb{E}\left[\frac{Y\mathbf{u}}{\pi(A, \mathbf{x})} \bigg| \mathbf{x}\right]}_{=:\mathbf{f}_{\text{opt}}(\mathbf{x})}, \tag{2}$$

where $\mathbf{u}$ is a random treatment vector in $\mathbb{R}^K$, corresponding to the random treatment $A$. Then estimating the optimal ITR can be converted into estimating the decision function $\mathbf{f}(\mathbf{x}) = (f_1(\mathbf{x}), \ldots, f_{K-1}(\mathbf{x}))^T$, assigning a $K-1$ dimensional vector to each covariate $\mathbf{x}$. Here, while the optimal decision function $\mathbf{f}_{\text{opt}}(\cdot)$ can take a generic form, we assume the linear decision function within the linear treatment rule class $\mathcal{D}$, induced by the linear decision function class $\mathcal{F} = \{\mathbf{f}(\mathbf{X}) = \mathbf{B}^T\mathbf{X} : \mathbf{B} \in \mathbb{R}^{p \times (K-1)}\}$.

**Reduction to Weighted Convex Optimization.** Suppose we have a sample $(\mathbf{x}_i, a_i, y_i)_{i=1}^n$ of size $n$ from the joint population distribution for $(\mathbf{X}, A, Y)$. Estimating the optimal decision function $\mathbf{f}_{\text{opt}}$ from the observed finite sample can be reduced to solving a weighted convex optimization problem:

$$\min_{\mathbf{B} \in \mathbb{R}^{p \times (K-1)}} \frac{1}{n} \sum_{i=1}^{n} w(a_i, \mathbf{x}_i)\, \ell(y_i, \mathbf{x}_i, a_i; \mathbf{B}) + R(\mathbf{B}). \tag{3}$$

There are three key components in the problem (3): the per-sample objective function $\ell$, the per-sample weights $w$, and the regularization $R(\mathbf{B})$. First, $\ell$ denotes a model-dependent per-sample convex objective function given by

$$\ell(y_i, \mathbf{x}_i, a_i; \mathbf{B}) = \begin{cases} \frac{1}{2}\left(\frac{K}{K-1}y_i - \mathbf{u}_{a_i}^T \mathbf{B}^T \mathbf{x}_i\right)^2 & \text{for continuous outcome,} \\ \log\left(1 + \exp(\mathbf{u}_{a_i}^T \mathbf{B}^T \mathbf{x}_i)\right) - y_i \mathbf{u}_{a_i}^T \mathbf{B}^T \mathbf{x}_i & \text{for binary outcome.} \end{cases} \quad (4)$$

An important characteristic of causal inference problems is that only one of the possible treatments can be observed for each subject. Thus, it is critical to control confounding variables to isolate the causal effect of the treatment from other factors that might influence the outcome. This effectively is done by choosing the appropriate weight $w_i = w(a_i, \mathbf{x}_i)$ for the $i$th subject. A classic approach is to use IPW: $w(a, \mathbf{x}) = 1/\pi(a, \mathbf{x})$ [39].

## 2.2 Proposed ITR Estimation Framework

**Reducing Finite Sample Bias.** The classic IPW approach requires specifying a propensity score model for confounding control. However, it is well-known that propensity score methods are highly sensitive to model misspecification, which yields biased estimates of causal effects [25, 30]. To address this issue, we investigate incorporating alternative, model-free weighting schemes such as *energy balancing weights* (EBWs) [23, 24], which are a type of DCBWs. Other options for DCBWs include *maximum mean discrepancy* (MMD) balancing weights [19, 27, 8] and Wasserstein distance-based balancing weights [52, 1]. Specifically, under the working model assumption (1), the optimal decision function $\mathbf{f}_{\text{opt}}$ in (2) can be decomposed as

$$\mathbf{f}_{\text{opt}}(\mathbf{x}) = \underbrace{\mu(\mathbf{x})\mathbb{E}\left[\frac{\mathbf{u}}{\pi(A, \mathbf{x})}\bigg|\mathbf{x}\right]}_{\text{treatment-free}} + \underbrace{\sum_{k=1}^{K}\delta_k(\mathbf{x})\mathbb{E}\left[\frac{\mathbf{u}I(A=k)}{\pi(A, \mathbf{x})}\bigg|\mathbf{x}\right]}_{\text{interaction}} + \mathbb{E}\left[\frac{\mathbf{u}}{\pi(A, \mathbf{x})}\bigg|\mathbf{x}\right]\mathbb{E}[\epsilon|\mathbf{x}]. \quad (5)$$

Since $\mathbb{E}\left[\frac{\mathbf{u}}{\pi(A,\mathbf{x})}|\mathbf{x}\right] = 0$ and $\mathbb{E}[\epsilon|\mathbf{x}] = 0$, it follows that $\mathbf{f}_{\text{opt}}(\mathbf{x}) = \sum_{k=1}^{K}\delta_k(\mathbf{x})\mathbf{u}_k$, depending only on the interaction term. However, suppose the true propensity score is not correctly estimated. In that case, the estimated decision function may be biased and include additional factors including the treatment-free effect, leading to sub-optimal treatment decisions. There are two possible sources for this estimation error. First, an incorrectly specified propensity score model might lead to an estimated propensity score $\hat{\mathbb{P}}_n(A=a|\mathbf{x})$ that deviates from the true propensity score $\pi(A=a|\mathbf{x})$. Second, even with correct specification, insufficient sample size may lead to inaccurate finite-sample approximation $\hat{\mathbb{P}}_n(A=a|\mathbf{x})$ of the true propensity score $\pi(A=a|\mathbf{x})$ [31]. It results in systematic bias for the estimated coefficient of the treatment-free term:

$$\hat{\mathbb{E}}_n\left[\frac{\mathbf{u}}{\pi(A, \mathbf{x})}\bigg|\mathbf{x}\right] = \sum_{k=1}^{K}\frac{\mathbf{u}_k}{\pi(k, \mathbf{x})}\hat{\mathbb{P}}_n(A=k|\mathbf{x}) \neq 0.$$

Instead of using IPW, we use DCBWs such as EBWs. Such weights are data-driven and do not rely on specific model assumptions or large sample approximation. Furthermore, EBWs are known to promote independence between treatment and confounders [23], so $\hat{\mathbb{E}}_n[\mathbf{u}w(A, \mathbf{x})\,|\,\mathbf{x}] \approx \hat{\mathbb{E}}_n[\mathbf{u}]\,\hat{\mathbb{E}}_n[w(A, \mathbf{x})\,|\,\mathbf{x}] = 0$ since $\hat{\mathbb{E}}_n[\mathbf{u}] = 0$. Thus, EBWs may reduce the unfavorable impacts of bias on the finite-sample decision function.

**Improving Optimization Guarantees.** To our best knowledge, the impact of the choice of the regularization $R(\mathbf{B})$ on the model parameter $\mathbf{B}$ on the detailed optimization landscape, especially in terms of the convergence rate and statistical estimation error bounds has been under-investigated in the literature. Contrary to the literature, we propose to use the combination of *soft $L_2$-regularization and a hard $L_1$-ball constraint:*.

$$\min_{\mathbf{B}\in\mathbb{R}^{p\times(K-1)},\,\|\mathbf{B}\|\leq\lambda_1}\left[\mathcal{L}(\mathbf{B}) := \frac{1}{n}\sum_{i=1}^{n}w(a_i, \mathbf{x}_i)\,\ell(y_i, x_i, a_i; \mathbf{B}) + \frac{\lambda_2}{2}\|\mathbf{B}\|_F^2\right]. \quad (6)$$

The above formulation has a smooth objective function that is unaffected by the hard $L_1$-ball constraint. Also, $L_2$-regularization ensures that the objective is at least $\lambda_2$-strongly convex. Thus, the

PGD algorithm (with suitable stepsize) enjoys a strong exponential convergence guarantee toward the global optimum. In contrast, standard approaches with $L_1$-penalization loss need sub-gradient methods since it is non-smooth. It is known that the convergence rate of the sub-gradient methods is much slower than gradient methods [6, 18].

We use the classical PGD to solve the convex-constrained optimization problem in (6). In each iteration, it involves conducting gradient descent with a suitable stepsize $\alpha_t$, followed by a projection $\Pi_{\mathcal{B}}$ onto the $L_1$-ball $\mathcal{B} := \{\mathbf{B} \in \mathbb{R}^{p \times (K-1)} : \|\mathbf{B}\| \leq \lambda_1\}$:

$$\mathbf{B}_t \leftarrow \Pi_{\mathcal{B}}\big(\mathbf{B}_{t-1} - \alpha_t \nabla_{\mathbf{B}} \mathcal{L}(\mathbf{B}_{t-1})\big). \tag{7}$$

For the projection onto the $L_1$-projection, we use the algorithm in [14]. Detailed implementation of this PGD algorithm is discussed in Algorithm 1 in the appendix.

**Additional Improvements by Variance and Dimension Reduction.** To ensure the stability of estimates for both balancing weights and outcome regression models, a variable screening step is recommended. We propose adapting distance covariance test [48] for its computational efficiency and its ability to detect complex dependencies between treatment and outcome. By focusing on screened variables that significantly impact the outcome, we can improve the estimation of outcome regression models. Additionally, estimating balancing weights based on effect modifiers rather than all variables provides more reliable weighting estimates. Hence, the screening approach serves to reduce bias and enhance the effectiveness of ITR-Learning. See Algorithm 2 in the appendix.

Lastly, we propose using both augmented outcomes and inverse variance weighting for variance reduction in learning ITRs. Under the working model assumption in (1), the interaction effect determines optimal treatment assignments, while the treatment-free effect acts as additional noise. Outcome augmentation helps reduce variance by mitigating residual variability due to the treatment-free effect. Nevertheless, misspecification of either the outcome or treatment-free effect models can induce heteroscedastic errors due to residual treatment-free effects [35]. In such scenarios, using inverse variance weighting, introduced in SABD-Learning, enhances robustness against misspecification [44]. Consequently, the combination of outcome augmentation and inverse variance weighting offers a robust and efficient approach to ITR-Learning, as they are complementary methods. Details are discussed in the appendix C.3 and C.4.

## 3 Statement of results

Our theoretical analysis shows that a "weighted design matrix", specifically a $p(K-1) \times p(K-1)$ positive semi-definite matrix, plays a central role in the local landscape analysis of the ITR estimation problem:

$$\mathbf{\Psi} := \frac{1}{n} \sum_{i=1}^{n} \omega(a_i, \mathbf{x}_i)(\mathbf{u}_{a_i} \mathbf{u}_{a_i}^T) \otimes \big(\mathbf{x}_i \mathbf{x}_i^T\big), \tag{8}$$

where $\otimes$ denotes the Kronecker product. Without loss of generality, we assume $\sum_i w(a_i, \mathbf{x}_i) = n$. Note that the weights $w_i = w(a_i, \mathbf{x}_i)$, $i = 1, \dots, n$ affect the eigenvalues of the weighted design matrix $\mathbf{\Psi}$. Our analysis reveals that, for the improved convergence rate of the PGD algorithm (Theorem 3.3) as well as a smaller statistical estimation error (Theorem 3.5), we need to choose the weights $w_i$ such that the minimum eigenvalue of $\mathbf{\Psi}$ is as large as possible. Roughly speaking, this is achieved when $w_1, \dots, w_n$ are 'covariate balancing weights' where the weighted sample covariance matrices conditional for each treatment are approximately the same. In the special case when the covariates are discrete and one-hot encoded, the optimal weights are the ones that exactly balance the covariate distributions given treatments. Thus this provides an explicit spectral characterization of the DCBWs in causal inference literature. See the appendix D for more discussions.

### 3.1 Computational Guarantees

In order to control the strong convexity and the smoothness parameter of the ITR objective (6), we require that the eigenvalues of $\mathbf{\Psi}$ are uniformly bounded.

**Assumption 3.1** (Eigenvalue bounds on the weighted design matrix). There are constants $0 \leq \lambda^- \leq \lambda^+$ such that the eigenvalues of $\mathbf{\Psi}$ in (8) is between $\lambda^-$ and $\lambda^+$,

For each subject $i = 1, \ldots, n$, let $z_i$ and $p_i$ denote its 'activation' and 'predictive probability', where

$$z_i := \mathbf{u}_{\mathbf{a}_i}^T \mathbf{B}^T \mathbf{x}_i, \qquad p_i := \exp(z_i)/(1 + \exp(z_i)). \tag{9}$$

We need the activation $z_i$ to be uniformly bounded for our theoretical analysis of the binary outcome. Since the simplex vectors $\mathbf{u}_{a_i}$ have unit length, it is enough to require the following assumption.

**Assumption 3.2** (Bounded activation). *There exists a constant $M > 0$ such that for all model parameter $\mathbf{B} \in \mathbb{R}^{p \times (K-1)}$ in the constraint set $\mathcal{B}$ and observed sample $i = 1, \ldots, n$, $\|\mathbf{B}^T \mathbf{x}_i\| \leq M$. Consequently, there exist constants $0 < \alpha^- \leq \alpha^+ \leq 1/4$ such that $\alpha^- \leq p_i(1 - p_i) \leq \alpha^+$ for all $i$.*

With our hard $L_1$-norm constraint on the model parameter $\mathbf{B}$, Assumption 3.2 can be easily verified whenever the covariate vectors $\mathbf{x}_i$ are uniformly bounded, which is a standard in the literature.

We establish the convergence rate of Algorithm 1 for ITR-Learning with stepsizes that are fixed but sufficiently small and diminishing rate. An informal statement is given in Theorem 3.3 below, and a full statement is provided in Theorem E.1 in the appendix. Define

$$\mu := \begin{cases} \lambda^- + \lambda_2 \\ \alpha^- \lambda^- + \lambda_2 \end{cases} \qquad L := \begin{cases} \lambda^+ + \lambda_2 & \text{for continuous outcome} \\ \alpha^+ \lambda^+ + \lambda_2 & \text{for binary outcome} \end{cases} \tag{10}$$

We say a function $f : \mathbb{R}^p \to \mathbb{R}$ is *$\mu$-strongly convex* if $f(x) - \frac{\mu}{2}\|x\|^2$ is convex and *$L$-smooth* if $f$ is differentiable and $\nabla f$ is $L$-Lipschitz continuous.

**Theorem 3.3** (Convergence rate of PGD for ITR-Learning). *Let $(\mathbf{B}_t)_{t \geq 0}$ denote the sequence of parameters obtained by the PGD algorithm (7) for ITR-Learning problem (6) with arbitrary initialization $\mathbf{B}_0$. Suppose Assumptions 3.1 and 3.2 hold. Let $\mathbf{B}^\star$ denote the unique global optimum of (6). Then the following hold:*

**(i)** *(Optimization landscape) The ITR objective $\mathcal{L}(\mathbf{B})$ in (6), is $\mu$-strongly convex and $L$-smooth, where $\mu, L$ are as in (10).*

**(ii)** *(Linear convergence with fixed stepsize) Assume constant stepsize $\alpha_t \equiv \alpha < 2/L$. Denote the contraction constant $\rho(\alpha) := \max\{|1 - \alpha L|, |1 - \alpha \mu|\} \in (0, 1)$. Then $\|\mathbf{B}_t - \mathbf{B}^\star\|_F^2 \leq \rho(\alpha)^t \|\mathbf{B}_0 - \mathbf{B}^\star\|_F^2$ for all $t \geq 1$.*

**(iii)** *(Sublinear convergence with diminishing stepsize) Assume diminishing stepsize $\alpha_t = \frac{\beta}{\gamma + t}$, where $\beta = \beta(\mu, L) > 0$ is a sufficiently large constant and $\gamma > 0$ is an arbitrary constant. Then $\|\mathbf{B}_t - \mathbf{B}^\star\|^2 \leq \frac{\nu}{\gamma + t}$ for all $t \geq 1$ for some constant $\nu > 0$.*

Our algorithm 1 with soft $L_2$-penalization and hard constraint with $L_1$-ball is guaranteed to converge to the true solution $\mathbf{B}^\star$ exponentially fast provided stepsizes $\alpha_t$ are fixed and sufficiently small.

## 3.2 Statistical Guarantees

Next, we introduce generative models for the ITR estimation problem, for which we will establish statistical estimation guarantees in Theorem 3.5. Fix a joint distribution $\pi$ for the pair $(\mathbf{X}, A)$ of covariates $\mathbf{X} \in \mathbb{R}^p$ and treatment $A \in \{1, \ldots, K\}$. Fix a true parameter $\mathbf{B}_\star \in \mathbb{R}^{p \times (K-1)}$. Then we assume $n$ i.i.d. samples $(\mathbf{x}_i, a_i, y_i)$ are drawn as $(\mathbf{x}_i, a_i) \sim \pi$ and

$$y_i | (\mathbf{x}_i, a_i) \sim \begin{cases} N\left(\frac{K-1}{K} z_i, \sigma^2\right) & \text{for continuous outcome} \\ \text{Ber}\left(\frac{\exp(z_i)}{1 + \exp(z_i)}\right) & \text{for binary outcome} \end{cases}, \quad \text{where } z_i := \mathbf{u}_{a_i}^T \mathbf{B}_\star^T \mathbf{x}_i, \tag{11}$$

where $N(\mu, \sigma^2)$ denotes a normal distribution with mean $\mu$ and variance $\sigma^2$, whereas $\text{Ber}(p)$ denotes a Bernoulli distribution with mean $p$. We assume that $\pi$ does not depend on $\mathbf{B}_\star$ and the noise variance $\sigma^2$ for the continuous case is known. We then seek to estimate the true parameter $\mathbf{B}_\star$ from the observed samples. It is easy to check that the negative log-likelihood function (up to additive constants) coincides with the per-sample loss function $\ell$ in (4). Therefore, the weighted convex optimization problem (3) corresponds to weighted MLE under the generative models in (11) (see Section H in the appendix for details).

We assume that the weighted gradient of the per-sample loss $\ell$ in (4) has uniformly bounded third moments for our statistical analysis of the generative ITR model above.

**Assumption 3.4** (Bounded moments at $\mathbf{B}_\star$ and weights). Suppose $(\mathbf{x}_i, a_i, y_i)$ follows the generative model above. Denote $U_i := w_i \nabla_\mathbf{B} \ell(y_i, \mathbf{x}_i, a_i; \mathbf{B}_\star) \in \mathbb{R}^{p \times (K-1)}$ for $w_i := w(a_i, \mathbf{x}_i)$ and $\overline{U}_i := U_i - \mathbb{E}[U_i]$. Suppose there are constants $D_1, d_1 \in (0, \infty)$ such that $\mathbb{E}[\|\overline{U}_i\|_F^3] < D_1$ and $\min_{k,l} \mathrm{Var}(\overline{U}_i(k,l)) > d_1$. Also, $\sum_{i=1}^n w_i^3 / \left( \sum_{i=1}^n w_i^2 \right)^{3/2} = O(n^{-1/2})$.

Now we state our main result regarding the statistical estimation guarantee for the true generative model parameter $\mathbf{B}_\star$. With high probability, Algorithm 1 can recover $\mathbf{B}_\star$ up to a statistical error $O(n^{-1/2})$.

**Theorem 3.5** (Statistical estimation guarantee). *Let* $(\mathbf{x}_i, a_i, y_i)_{i=1}^n$ *be i.i.d. observations from the generative models in* (11) *with true parameter* $\mathbf{B}_\star \in \mathbb{R}^{p \times (K-1)}$ *such that* $\|\mathbf{B}_\star\|_1 \leq \lambda_1$ *for some* $\lambda_1 \geq 0$. *Suppose Assumptions 3.1, 3.2, and 3.4 hold. Let* $\widehat{\mathbf{B}}_T$ *denote the weighted MLE obtained after $T$ iterations of PGD algorithm* (7) *for* (3). *Fix a constant* $\varepsilon > 0$.

*Then there exists a constant* $C = C(\varepsilon) > 0$ *such that with probability at least* $1 - \varepsilon$ *and for $\mu$ as in* (10) *but with* $\lambda^\pm = \lambda^\pm(\mathbb{E}[\boldsymbol{\Psi}])$,

$$\|\mathbf{B}_\star - \widehat{\mathbf{B}}_T\|_F \leq \frac{C\sqrt{(p/n)\log \epsilon^{-1}}}{\mu} + \frac{8\lambda_2 \|\mathbf{B}_\star\|_F}{\mu}, \tag{12}$$

*provided $n, T$ are large enough. More specifically, the above holds when*

**(i)** *(Sample complexity) $n \geq 1$ large s.t.* $\frac{(\sum_{i=1}^n w_i^2)^3}{(\sum_{i=1}^n w_i^3)^2} \geq C_1 \varepsilon^{-2}$ *for some explicit constant $C_1 > 0$;*

**(ii)** *(Computational complexity) $T = O(\log n)$ and $O(n^{-1})$ when constant or diminishing stepsize as in Theorem 3.3 is used, respectively.*

*Furthermore, we get $\sqrt{n}$-consistent estimation whenever* $\lambda_2 = O(n^{-1/2}\|\mathbf{B}_\star\|_F^{-1})$.

Notice that both terms in the error bound (12) are proportional to $1/\mu$, which depends both on the minimum eigenvalue $\lambda^-$ of $\mathbb{E}[\boldsymbol{\Psi}]$ as well as the $L_2$-regularization parameter $\lambda_2$. Therefore, choosing the weighting function $\omega(A, \mathbf{X})$ to maximize $\lambda^-$ helps minimize the overall statistical estimation error. (Detailed analysis and discussion on this point can be found in the appendix H.) Another easy way to increase $\mu$ is to use large $L_2$-regularization, but as in the second term in (12), using $L_2$-regularization gives estimation bias of $O(\lambda_2 \|\mathbf{B}_\star\|_F / \mu)$. Hence, we need to choose $\lambda_2$ is small enough so that this bias term is also of $O(n^{-1/2})$. Then we obtain an overall $\sqrt{n}$-consistent estimator.

The key insight in our proof of Theorem 3.5 is that we can decompose the total estimation error $\|\mathbf{B}_\star - \widehat{\mathbf{B}}_T\|_F$ into computational and statistical parts:

$$\|\mathbf{B}_\star - \widehat{\mathbf{B}}_T\|_F \leq \underbrace{\|\widehat{\mathbf{B}} - \widehat{\mathbf{B}}_T\|_F}_{\text{computational error}} + \underbrace{\|\mathbf{B}_\star - \widehat{\mathbf{B}}\|_F}_{\text{statistical error}}, \tag{13}$$

where $\widehat{\mathbf{B}}$ denotes the exact MLE. The computational error vanishes as the number $T$ of PGD iterations tends to infinity according to Theorem 3.3. For the statistical error, we show

$$\mathbb{P}\left( \|\mathbf{B}_\star - \widehat{\mathbf{B}}\|_F \leq \frac{2C}{\sqrt{n}} + \frac{8\lambda_2 \|\mathbf{B}_\star\|_F}{\mu} \right) \geq 1 - O\left( \frac{\sum_{i=1}^n w_i^3}{\left(\sum_{i=1}^n w_i^2\right)^{3/2}} \right). \tag{14}$$

Thus, the 'skewness' of the weights $w_i$ measured by the ratio $\sqrt{n} \sum_{i=1}^n w_i^3 / \left(\sum_{i=1}^n w_i^2\right)^{3/2}$ (which arises from Berry-Esseen bound) acts as a scalar multiple to the standard statistical error of $O(n^{-1/2})$ per the central limit theorem. Hence, more balanced weights lead to fewer statistical estimation errors. If this skewness ratio is uniformly bounded, we can ensure a small statistical error with probability at least $1 - \varepsilon$, provided we have at least $O(\varepsilon^{-2})$ samples. Then, we simply need to ensure the PGD iteration $T$ is large enough so that the computational error is at most $C/\sqrt{n}$ to achieve the high-probability total estimation error bound in Theorem 3.5. Another insight from the theorem is that maximizing $\lambda_{\min}(\mathbb{E}[\boldsymbol{\Psi}])$ promotes heterogeneous balancing weights $w_i$, and we get increased sample complexity through the bound $\frac{(\sum_{i=1}^n w_i^2)^3}{(\sum_{i=1}^n w_i^3)^2} \geq C_1 \varepsilon^{-2}$ (e.g., if $w_i \equiv 1$, this is $n \gtrsim \epsilon^{-2}$, but if $w_1 = n, w_2 = \cdots = w_n = 0$, this may not be satisfied for any $n$).

# 4 Simulation Studies

We investigate four simulation settings, each designed to evaluate various factors influencing the performance of different ITR estimators. For each setting, we mimic a randomized trial (no-confounding) and an observational study (with confounding) as per [44]. We consider 4 treatments with sample sizes $n = 200, 600, 1000$, and covariate dimensions $p = 20, 40, 60$. The outcome $Y$ follows the model (1) with $\epsilon | A, \mathbf{X} \sim N\big(0, \sigma^2(A, \mathbf{X})\big)$. We evaluate method performance using accuracy (i.e., correct identification of the optimal treatment for each observation) and empirical value on a test dataset of $10,000$ observations. Each simulation setting is replicated independently 100 times.

In this section, we provide partial experiment results according to the proposed estimation framework in Section 2.2. Specifically, we focus on the results from two scenarios: one mimicking randomized trials with linear ITR as the optimal rule and the other mimicking observational studies with nonlinear ITR as the optimal rule for continuous outcome. Since our proposed method assumes linear treatment rule class, the latter involves a situation where our model's prespecified treatment rule class is misspecified. Specifically, we use following treatment-free effect function $\mu(\mathbf{X})$ and interaction effect function $\delta(\mathbf{X})$ for each scenario:

1. Randomized Trial: Linear ITR as the true optimal
$\mu(\mathbf{X}) = 1 + 2X_1 + 2X_2,$

$$\delta(\mathbf{X}) = \begin{cases} 0.75 + 1.5X_1 + 1.5X_2 + 1.5X_3 + 1.5X_4, A = 1; 0.75 + 1.5X_1 - 1.5X_2 - 1.5X_3 + 1.5X_4, A = 2; \\ 0.75 + 1.5X_1 - 1.5X_2 + 1.5X_3 - 1.5X_4, A = 3; 0.75 - 1.5X_1 + 1.5X_2 - 1.5X_3 - 1.5X_4, A = 4, \end{cases}$$

2. Observational Study: Nonlinear ITR as the true optimal
$\mu(\mathbf{X}) = 1 + 2X_1 + 2X_2 + 2X_4 - 2X_4^2 + 2X_1X_2 + 2e^{-X_1X_2} + \sin(X_3),$

$$\delta(\mathbf{X}) = \begin{cases} 0.5 + 1.0X_1 - 2.0X_4 + 0.5X_4^2, A = 1; \ 1.0 + 1.0X_1 + 1.0X_4 - 1.0X_4^2, A = 2; \\ 1.5 + 2.0X_1 - 1.0X_4 - 1.0X_4^2, A = 3; \ 1.0 - 1.0X_1 - 1.0X_4 - 1.0X_4^2, A = 4. \end{cases}$$

in the working model (1). Further details about simulation settings, other scenarios, and corresponding results are available in the appendix I.

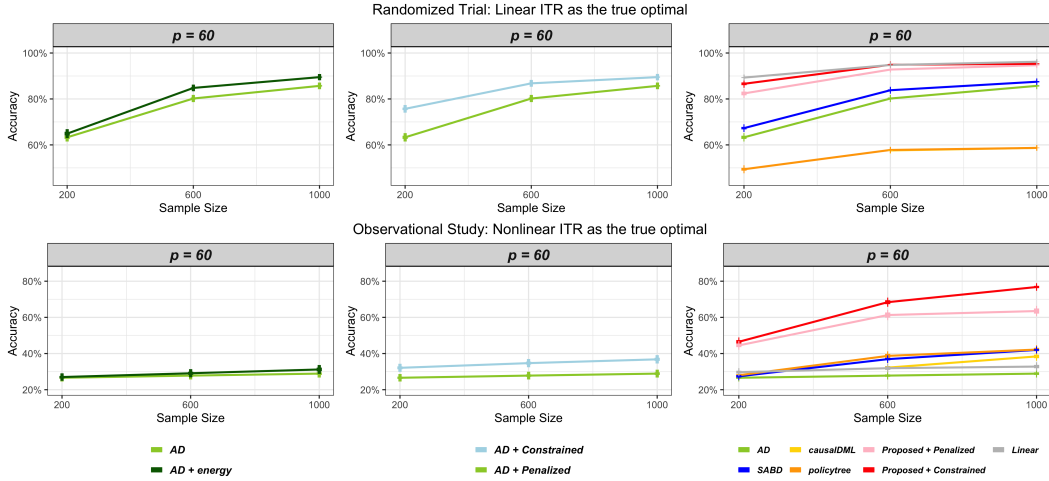

Figure 1: **Accuracy Comparison: proposed methods vs. benchmark methods.** All subplots in the same row share the same simulation setting, focusing on randomized trials with linear ITR as the true optimal rule (top) and observational studies with nonlinear ITR as the true optimal rule (bottom). Each subplot presents (Left) accuracy comparisons based on weights, illustrating the difference between the standard IPW approach of AD-Learning with the proposed approach using EBW, (Middle) accuracy comparisons based on optimization algorithms, illustrating the difference between the standard $L_1$-penalized approach against the proposed constrained optimization with PGD, (Right) evaluation of accuracies between existing standard approaches and the proposed method, which integrates EBWs, variance and dimension reduction techniques implemented through PGD. Error bars represent the standard errors of the mean (SEM) of accuracies across multiple simulations.

**Using Distributional Covariate Balancing Weights.** In the left subplots of Figure 1, the incorporation of DCBWs (e.g., EBWs) shows improvements in classification rates for ITR-Learning. While EBWs indeed provide more effective balancing weights in observational studies, as depicted in the lower left subplot in Figure 1, it also results in improvements even in randomized trial settings where the true propensity score is known. This is achieved by reducing finite sample imbalances, as shown in the upper left subplot in Figure 1. Moreover, when the underlying true treatment rule class is nonlinear, the performance of AD-Learning suffers due to a misalignment between the assumed linear treatment rule class and the actual class. In this scenario, using EBWs yields accuracy improvements compared to the conventional IPW approach. Further details can be found in the appendix I.

**Using PGD for Constrained Optimization.** The middle subplots of Figure 1 show accuracy comparisons between standard penalized approach and constrained optimization using PGD. Both optimization algorithms are based on the same proposed statistical approach, integrating EBWs with SABD-Learning, outcome augmentation, and additional variable screening. For solving the $L_1$-penalized regression problems, the R package *glmnet* is a standard choice, which is based on cyclic coordinate descent with simulated-annealing-style hyperparameter tuning. However, *glmnet* does not handle hard $L_1$-ball constraints required for our method. Therefore, to ensure a fair comparison, we directly implemented standard (projected) (sub)gradient descent to solve optimization problems for both penalized and constrained problems. In our experiment reported in Figure 1, we observe significant improvements using our PGD method, especially in the second scenario. One possible explanation for this improvement is that large penalization may yield a perturbed solution to the ITR problem. If the minimizer of the ITR problem lies on the boundary of the $L_1$-ball, the solutions of constrained optimization and penalized optimization coincide. However, if the minimizer is the interior point of $L_1$-ball, constrained optimization can find the exact solution, while the penalization may induce a perturbed solution. Furthermore, the trajectory of PGD for the constraint approach appears to be significantly more stable than that of the subgradient descent in the penalization approach (see Figure 2).

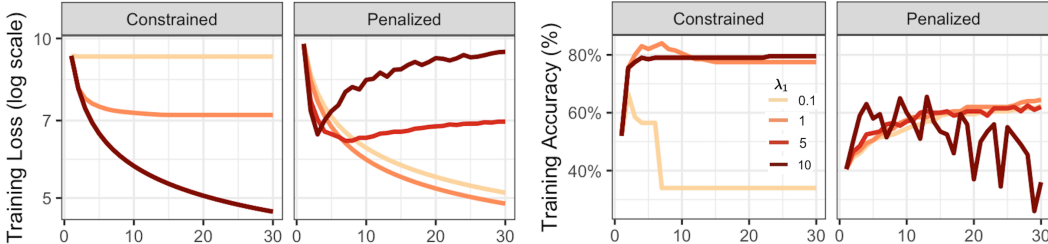

Figure 2: **Comparison of Optimization Methods by $\lambda_1$.** $\lambda_1$ is $L_1$-ball size of the constraint set in the constrained optimization and the regularization parameter with additional $L_1$-regularization of the model parameter in the penalized regression, respectively.

**Performance Evaluations between the Proposed Method and Benchmark Methods.** In the right subplots of Figure 1, we evaluate accuracies by comparing benchmark methods to our proposed approach. The benchmarks include AD-Learning, SABD-Learning, a treatment-covariate interaction model with $L_1$-regularization 'linear', and two popular tree-based methods:'policytree' [47, 59] and double-machine learning-based 'causalDML' [10, 29]. Our proposed method integrates EBWs with SABD-Learning, outcome augmentation, and additional variable screening for variance and dimension reduction, implemented via PGD. However, causalDML requires a sufficiently large sample size, thereby making its evaluation unfeasible under a sample size of 200. Our numerical analysis provides evidence that our proposed approach outperforms existing methods in both scenarios. 'linear' performs well in the upper subplot, which simulates a randomized trial with a true optimal linear rule, as covariates are balanced and the model accurately captures the decision function. However, in the lower subplot, simulating an observational study with a nonlinear rule, the model performs poorly because it lacks balancing weights to address confounding and cannot fully capture nonlinear relationships. Further detailed analysis, including the effectiveness of additional techniques such as variable screening for performance improvement, is provided in the appendix I.

Additionally, our proposed method processes each dataset (p = 60, n = 1000) in an average of 4.40 seconds, compared to 2.01 seconds for the penalization method using *glmnet* in R. Other methods exhibit scalability challenges, with policytree averaging 12.02 seconds and causalDML taking 46.75 seconds. This demonstrates that computational efficiency is a notable advantage of our approach.

# 5 Applications

We apply the proposed methods to two datasets from AIDS Clinical Trials Group (ACTG) 175 [21] and email marketing [22]. ACTG is a randomized trial for 2,139 patients with HIV infection who were randomly assigned to four different treatments. Twelve covariates including age and gender were used for the analysis. The outcome is the change in CD4 cell count from baseline to 20 weeks, where larger values are preferable. Email marketing dataset has 64,000 customers who were randomly chosen to receive an e-mail campaign among three different marketing methods. Eight covariates with historical customer attributes and their pairwise interactions were used. The outcome is whether customers visited the website in the following two weeks. The goal of the analysis using these two datasets is to find the best treatment/marketing methods based on individuals' attributes. Similar to [44], we randomly split the data into a training set of $\{200, 400, 800, 1000, 1200\}$ observations for the ACTG dataset and $\{1000, 3000, 5000\}$ observations for the email dataset. The remaining observations were used for test data with 10 iterations. We evaluate method performance using the empirical value function on a test dataset. The benchmarking methods include AD-Learning, SABD-Learning, and two tree-based methods (policytree [47] and causalDML [29]). However, causalDML requires a sufficiently large sample size, thereby making its evaluation unfeasible under a sample size of 200. For the binary outcome in Email dataset, SABD-Learning is not used, since the homoscedastic assumption is unnecessary for logistic regression. The results are shown in Table 1.

Table 1: Average empirical value functions across different approaches for ACTG/Email datasets. Mean values with the corresponding standard errors of the mean (SEM) in parentheses are provided. The highest-performing methods are marked in bold.

| Dataset | Training size | AD | SABD | policytree | causalDML | **Proposed** |
|---------|--------------|-----|------|-----------|-----------|-----------|
| ACTG | 200 | 37.624 (6.9) | 33.645 (6.7) | 36.871 (4.6) | - | **42.301** (5.7) |
| | 400 | 51.329 (3.2) | 53.644 (1.9) | 47.191 (2) | 49.552 (0.2) | **54.868** (3.4) |
| | 800 | 56.021 (1.5) | 56.700 (1.6) | 52.786 (2.6) | 55.506 (2.1) | **59.200** (2.5) |
| | 1000 | 56.825 (2.2) | 56.963 (2.1) | 56.005 (1.7) | 56.591 (2.1) | **59.552** (2.5) |
| | 1200 | 55.092 (2.6) | 55.221 (2.8) | **58.819** (3.4) | 57.104 (2.8) | 58.207 (2.9) |
| Email | 1000 | 0.169 (0.0) | - | 0.171 (0.0) | 0.172 (0.0) | **0.175** (0.0) |
| | 3000 | 0.176 (0.0) | - | 0.177 (0.0) | 0.177 (0.0) | **0.181** (0.0) |
| | 5000 | 0.177 (0.0) | - | 0.178 (0.0) | 0.178 (0.0) | **0.182** (0.0) |

# 6 Discussion and Limitations

Our method introduces a unified, robust framework for estimating ITR. By formulating the problem as a weighted, constrained optimization problem, we incorporate DCBWs to control confounding and propose the PGD algorithm for sparse ITRs with computational and statistical guarantees. Additionally, we propose variable screening and efficient augmentation, which together show synergistic effects. We demonstrate our method with continuous and binary outcomes, and it can be readily extended to other outcomes, such as censored outcomes. Although we focus on learning linear ITR to facilitate the demonstration of our idea, it can be extended to nonlinear ITR using the basis function [34]. For example, by taking the covariate powers up to M, the linear function class becomes the class of degree-M polynomial functions. The linear function class can also serve as a useful approximation to nonlinear decision functions.

We acknowledge that although most, if not all, types of ITR-Learning frameworks can benefit from using DCBWs, variable screening, and augmentation, our proposed framework for theoretical analysis relies on the specific assumption that the model can be formulated as a constrained, weighted, and convex optimization problem. Furthermore, we restricted the decision function to the linear function class. Generalizing our theoretical analysis to include nonlinear function classes with nonconvex problems, such as boosting or deep neural networks, is an interesting direction for future investigation. Another limitation of our method is its reliance on standard assumptions to identify optimal ITRs using observational data, including the assumption of no unmeasured confounding. An interesting future direction would be to extend our framework to cases where this assumption does not hold, utilizing instrumental variables [12, 38] and proximal causal learning [40, 45].

Our work significantly impacts society by improving patient health outcomes through personalized care and advancing medical research in disease mechanisms and drug development. Beyond healthcare, these methods can be applied in marketing strategies based on customer characteristics.

## Acknowledgement

The authors would like to thank Dr. Michael Kosorok and Dr. Haoda Fu for their valuable comments and encouragement on the early version of this work. Lee J. and Chen G.'s efforts were partially supported by NSF Grant DMS-2054346.

## References

[1] Serge Assaad, Shuxi Zeng, Chenyang Tao, Shounak Datta, Nikhil Mehta, Ricardo Henao, Fan Li, and Lawrence Carin. Counterfactual representation learning with balancing weights. In *International Conference on Artificial Intelligence and Statistics*, pages 1972–1980. PMLR, 2021.

[2] Susan Athey and Stefan Wager. Policy learning with observational data. *Econometrica*, 89(1):133–161, 2021.

[3] Laura B Balzer, Maya L Petersen, Mark J van der Laan, and Search Collaboration. Targeted estimation and inference for the sample average treatment effect in trials with and without pair-matching. *Statistics in medicine*, 35(21):3717–3732, 2016.

[4] Léon Bottou. Large-scale machine learning with stochastic gradient descent. In *Proceedings of COMPSTAT'2010*, pages 177–186. Springer, 2010.

[5] Stephen Boyd, Stephen P Boyd, and Lieven Vandenberghe. *Convex optimization*. Cambridge university press, 2004.

[6] Stephen Boyd, Lin Xiao, and Almir Mutapcic. Subgradient methods. *lecture notes of EE392o, Stanford University, Autumn Quarter*, 2004(01), 2003.

[7] Guanhua Chen, Donglin Zeng, and Michael R Kosorok. Personalized dose finding using outcome weighted learning. *J. Amer. Statist. Assoc.*, 111(516):1509–1521, 2016.

[8] Rui Chen, Jared D Huling, Guanhua Chen, and Menggang Yu. Robust sample weighting to facilitate individualized treatment rule learning for a target population. *Biometrika*, 111(1):309–329, 2024.

[9] Shuai Chen, Lu Tian, Tianxi Cai, and Menggang Yu. A general statistical framework for subgroup identification and comparative treatment scoring. *Biometrics*, 73(4):1199–1209, 2017.

[10] Victor Chernozhukov, Denis Chetverikov, Mert Demirer, Esther Duflo, Christian Hansen, and Whitney Newey. Double/debiased/neyman machine learning of treatment effects. *Am. Econ. Rev.*, 107(5):261–65, 2017.

[11] Francis S Collins and Harold Varmus. A new initiative on precision medicine. *New England journal of medicine*, 372(9):793–795, 2015.

[12] Yifan Cui and Eric Tchetgen Tchetgen. A semiparametric instrumental variable approach to optimal treatment regimes under endogeneity. *Journal of the American Statistical Association*, 116(533):162–173, 2021.

[13] Alicia Curth and Mihaela van der Schaar. Nonparametric estimation of heterogeneous treatment effects: From theory to learning algorithms. In *International Conference on Artificial Intelligence and Statistics*, pages 1810–1818. PMLR, 2021.

[14] John Duchi, Shai Shalev-Shwartz, Yoram Singer, and Tushar Chandra. Efficient projections onto the l 1-ball for learning in high dimensions. In *Proceedings of the 25th international conference on Machine learning*, pages 272–279, 2008.

[15] Jianqing Fan and Runze Li. Variable selection via nonconcave penalized likelihood and its oracle properties. *Journal of the American statistical Association*, 96(456):1348–1360, 2001.

[16] Jianqing Fan and Jinchi Lv. Sure independence screening for ultrahigh dimensional feature space. *Journal of the Royal Statistical Society: Series B (Statistical Methodology)*, 70(5):849–911, 2008.

[17] William Feller. *An introduction to probability theory and its applications, Volume 2*, volume 81. John Wiley & Sons, 1991.

[18] Jean-Louis Goffin. On convergence rates of subgradient optimization methods. *Mathematical programming*, 13:329–347, 1977.

[19] Arthur Gretton, Karsten M Borgwardt, Malte J Rasch, Bernhard Schölkopf, and Alexander Smola. A kernel two-sample test. *The Journal of Machine Learning Research*, 13(1):723–773, 2012.

[20] Jens Hainmueller. Entropy balancing for causal effects: A multivariate reweighting method to produce balanced samples in observational studies. *Political analysis*, 20(1):25–46, 2012.

[21] Scott M Hammer, David A Katzenstein, Michael D Hughes, Holly Gundacker, Robert T Schooley, Richard H Haubrich, W Keith Henry, Michael M Lederman, John P Phair, Manette Niu, et al. A trial comparing nucleoside monotherapy with combination therapy in hiv-infected adults with cd4 cell counts from 200 to 500 per cubic millimeter. *New England Journal of Medicine*, 335(15):1081–1090, 1996.

[22] Kevin Hillstrom. Minethatdata e-mail analytics and data mining challenge. https://www.uplift-modeling.com/en/v0.3.1/api/datasets/fetch_hillstrom.html, 2008. Accessed: 2024-04-28.

[23] Jared D Huling, Noah Greifer, and Guanhua Chen. Independence weights for causal inference with continuous treatments. *Journal of the American Statistical Association*, 119(546):1657–1670, 2024.

[24] Jared D Huling and Simon Mak. Energy balancing of covariate distributions. *Journal of Causal Inference*, 12(1):20220029, 2024.

[25] Kosuke Imai and Marc Ratkovic. Covariate balancing propensity score. *Journal of the Royal Statistical Society: Series B (Statistical Methodology)*, 76(1):243–263, 2014.

[26] Nathan Kallus. Balanced policy evaluation and learning. *Advances in neural information processing systems*, 31, 2018.

[27] Nathan Kallus. Generalized optimal matching methods for causal inference. *The Journal of Machine Learning Research*, 21(1):2300–2353, 2020.

[28] Nathan Kallus. More efficient policy learning via optimal retargeting. *Journal of the American Statistical Association*, 116(534):646–658, 2021.

[29] Michael C Knaus. Double machine learning-based programme evaluation under unconfoundedness. *The Econometrics Journal*, 25(3):602–627, 2022.

[30] Brian K Lee, Justin Lessler, and Elizabeth A Stuart. Weight trimming and propensity score weighting. *PloS one*, 6(3):e18174, 2011.

[31] Fan Li, Kari Lock Morgan, and Alan M Zaslavsky. Balancing covariates via propensity score weighting. *Journal of the American Statistical Association*, 113(521):390–400, 2018.

[32] Runze Li, Wei Zhong, and Liping Zhu. Feature screening via distance correlation learning. *Journal of the American Statistical Association*, 107(499):1129–1139, 2012.

[33] Haixu Ma, Donglin Zeng, and Yufeng Liu. Learning individualized treatment rules with many treatments: A supervised clustering approach using adaptive fusion. *Advances in Neural Information Processing Systems*, 35:15956–15969, 2022.

[34] Jacob M Maronge, Jared D Huling, and Guanhua Chen. A reluctant additive model framework for interpretable nonlinear individualized treatment rules. *The Annals of Applied Statistics*, 17(4):3384–3402, 2023.

[35] Weibin Mo and Yufeng Liu. Efficient learning of optimal individualized treatment rules for heteroscedastic or misspecified treatment-free effect models. *Journal of the Royal Statistical Society Series B: Statistical Methodology*, 84(2):440–472, 2022.

[36] Xinkun Nie and Stefan Wager. Quasi-oracle estimation of heterogeneous treatment effects. *Biometrika*, 108(2):299–319, 2021.

[37] Wenliang Pan, Xueqin Wang, Heping Zhang, Hongtu Zhu, and Jin Zhu. Ball covariance: A generic measure of dependence in banach space. *Journal of the American Statistical Association*, 115(529):307–317, 2019.

[38] Hongming Pu and Bo Zhang. Estimating optimal treatment rules with an instrumental variable: A partial identification learning approach. *Journal of the Royal Statistical Society Series B: Statistical Methodology*, 83(2):318–345, 2021.

[39] Zhengling Qi, Dacheng Liu, Haoda Fu, and Yufeng Liu. Multi-armed angle-based direct learning for estimating optimal individualized treatment rules with various outcomes. *Journal of the American Statistical Association*, 115(530):678–691, 2020.

[40] Zhengling Qi, Rui Miao, and Xiaoke Zhang. Proximal learning for individualized treatment regimes under unmeasured confounding. *Journal of the American Statistical Association*, pages 1–14, 2023.

[41] Min Qian and Susan A Murphy. Performance guarantees for individualized treatment rules. *Annals of statistics*, 39(2):1180, 2011.

[42] Brian C Ross. Mutual information between discrete and continuous data sets. *PloS one*, 9(2):e87357, 2014.

[43] Donald B Rubin. Randomization analysis of experimental data: The fisher randomization test comment. *Journal of the American statistical association*, 75(371):591–593, 1980.

[44] Kushal S Shah, Haoda Fu, and Michael R Kosorok. Stabilized direct learning for efficient estimation of individualized treatment rules. *Biometrics*, 2022. doi: 10.1111/biom.13818.

[45] Tao Shen and Yifan Cui. Optimal treatment regimes for proximal causal learning. *Advances in Neural Information Processing Systems*, 36, 2024.

[46] Claudia Shi, David Blei, and Victor Veitch. Adapting neural networks for the estimation of treatment effects. *Advances in neural information processing systems*, 32, 2019.

[47] Erik Sverdrup, Ayush Kanodia, Zhengyuan Zhou, Susan Athey, and Stefan Wager. policytree: Policy learning via doubly robust empirical welfare maximization over trees. *Journal of Open Source Software*, 5(50):2232, 2020.

[48] Gábor J Székely, Maria L Rizzo, and Nail K Bakirov. Measuring and testing dependence by correlation of distances. *The Annals of Statistics*, 35(6):2769–2794, 2007.

[49] Dingke Tang, Dehan Kong, Wenliang Pan, and Linbo Wang. Ultra-high dimensional variable selection for doubly robust causal inference. *Biometrics*, 79(2):903–914, 2023.

[50] Lu Tian, Ash A Alizadeh, Andrew J Gentles, and Robert Tibshirani. A simple method for estimating interactions between a treatment and a large number of covariates. *Journal of the American Statistical Association*, 109(508):1517–1532, 2014.

[51] Roman Vershynin. *High-dimensional probability: An introduction with applications in data science*, volume 47. Cambridge university press, 2018.

[52] Cédric Villani et al. *Optimal transport: old and new*, volume 338. Springer, 2009.

[53] Stefan Wager and Susan Athey. Estimation and inference of heterogeneous treatment effects using random forests. *Journal of the American Statistical Association*, 113(523):1228–1242, 2018.

[54] Baqun Zhang, Anastasios A Tsiatis, Eric B Laber, and Marie Davidian. A robust method for estimating optimal treatment regimes. *Biometrics*, 68(4):1010–1018, 2012.

[55] Chong Zhang and Yufeng Liu. Multicategory angle-based large-margin classification. *Biometrika*, 101(3):625–640, 2014.

[56] Qingyuan Zhao, Dylan S Small, and Ashkan Ertefaie. Selective inference for effect modification via the lasso. *Journal of the Royal Statistical Society Series B: Statistical Methodology*, 84(2):382–413, 2022.

[57] Yingqi Zhao, Donglin Zeng, A John Rush, and Michael R Kosorok. Estimating individualized treatment rules using outcome weighted learning. *Journal of the American Statistical Association*, 107(499):1106–1118, 2012.

[58] Xin Zhou, Nicole Mayer-Hamblett, Umer Khan, and Michael R Kosorok. Residual weighted learning for estimating individualized treatment rules. *Journal of the American Statistical Association*, 112(517):169–187, 2017.

[59] Zhengyuan Zhou, Susan Athey, and Stefan Wager. Offline multi-action policy learning: Generalization and optimization. *Operations Research*, 71(1):148–183, 2023.

[60] Ruoqing Zhu, Ying-Qi Zhao, Guanhua Chen, Shuangge Ma, and Hongyu Zhao. Greedy outcome weighted tree learning of optimal personalized treatment rules. *Biometrics*, 73(2):391–400, 2017.

[61] José R Zubizarreta. Stable weights that balance covariates for estimation with incomplete outcome data. *Journal of the American Statistical Association*, 110(511):910–922, 2015.

# A Preliminaries

In this section, we review several notations and basic facts on linear algebra and matrix calculus.

The Kronecker product, denoted by $\otimes$, is a binary operation that combines two matrices to create a new matrix. Given two matrices $\mathbf{A} = (a_{ij}) \in \mathbb{R}^{m \times n}$ and $\mathbf{B} \in \mathbb{R}^{p \times q}$, then $\mathbf{A} \otimes \mathbf{B}$ is an $mp \times nq$ matrix defined as

$$\mathbf{A} \otimes \mathbf{B} = \begin{bmatrix} a_{11}\mathbf{B} & \cdots & a_{1n}\mathbf{B} \\ \vdots & \ddots & \vdots \\ a_{m1}\mathbf{B} & \cdots & a_{mn}\mathbf{B} \end{bmatrix} \tag{15}$$

If $\mathbf{A}' \in \mathbb{R}^{m \times n'}$, then for the horizontally stacked matrix $[\mathbf{A}, \mathbf{A}'] \in \mathbb{R}^{m \times (n+n')}$,

$$[\mathbf{A}, \mathbf{A}'] \otimes \mathbf{B} = [\mathbf{A} \otimes \mathbf{B}, \mathbf{A}' \otimes \mathbf{B}]. \tag{16}$$

For each $\mathbf{A} = [\mathbf{a}_1, \ldots, \mathbf{a}_n] \in \mathbb{R}^{m \times n}$, let $\mathrm{vec}(\mathbf{A}) := [\mathbf{a}_1^T, \ldots, \mathbf{a}_n^T]^T \in \mathbb{R}^{mn}$ denote its vectorization. The *commutation matrix* $\mathbf{C}^{(a,b)}$, which is a special instance of $ab \times ab$ permutation matrix. Namely, for each integers $a, b \geq 1$, there exists a unique matrix $\mathbf{C}^{(a,b)} \in \{0,1\}^{ab \times ab}$ such that for all $\mathbf{A} \in \mathbb{R}^{a \times b}$, we have $\mathbf{C}^{(a,b)} \mathrm{vec}(\mathbf{A}) = \mathrm{vec}(\mathbf{A}^T)$. Note that $(\mathbf{C}^{(a,b)})^T = \mathbf{C}^{(b,a)}$. Furthermore, $(\mathbf{C}^{(a,b)})^T \mathbf{C}^{(a,b)} = \mathbf{I}_{ab}$ since $(\mathbf{C}^{(a,b)})^T \mathbf{C}^{(a,b)} \mathrm{vec}(\mathbf{A}) = \mathbf{C}^{(b,a)} \mathrm{vec}(\mathbf{A}^T) = \mathrm{vec}(\mathbf{A})$. Hence $\mathbf{C}^{(a,b)}$ is positive semi-definite. Note that $\mathbf{C}^{(a,1)} = \mathbf{I}_a = \mathbf{C}^{(1,a)}$ since if $\mathbf{A} \in \mathbb{R}^{a \times 1}$ or $\mathbf{A} \in \mathbb{R}^{1 \times a}$, $\mathrm{vec}(\mathbf{A}) = \mathrm{vec}(\mathbf{A}^T)$. Also, for every matrices $\mathbf{A} \in \mathbb{R}^{m \times n}$ and $\mathbf{B} \in \mathbb{R}^{r \times q}$,

$$\mathbf{C}^{(r,m)}(\mathbf{A} \otimes \mathbf{B}) = (\mathbf{B} \otimes \mathbf{A})\mathbf{C}^{(q,n)}. \tag{17}$$

Also recall the relations for vectorizing product of matrices: for $\mathbf{A} \in \mathbb{R}^{a \times b}$, $\mathbf{B} \in \mathbb{R}^{b \times c}$, and $\mathbf{C} \in \mathbb{R}^{c \times d}$,

$$\mathrm{vec}(\mathbf{AB}) = (\mathbf{I}_c \otimes \mathbf{A}) \mathrm{vec}(\mathbf{B}) = (\mathbf{B}^T \otimes \mathbf{I}_a) \mathrm{vec}(\mathbf{A}), \tag{18}$$

$$\mathrm{vec}(\mathbf{ABC}) = (\mathbf{C}^T \otimes \mathbf{A}) \mathrm{vec}(\mathbf{B}) = (\mathbf{I}_d \otimes \mathbf{AB}) \mathrm{vec}(\mathbf{C}) = (\mathbf{C}^T \mathbf{B}^T \otimes \mathbf{I}_a) \mathrm{vec}(\mathbf{A}). \tag{19}$$

For $\mathbf{A}, \mathbf{B}, \mathbf{C}$, and $\mathbf{D}$ are matrices of compatible sizes allowing the matrix products $\mathbf{AC}$ and $\mathbf{BD}$,

$$(\mathbf{A} \otimes \mathbf{B})(\mathbf{C} \otimes \mathbf{D}) = (\mathbf{AC} \otimes \mathbf{BD}).$$

Next, recall that if $f : \mathbb{R}^{a \times 1} \to \mathbb{R}^{b \times 1}$ and $g : \mathbb{R}^{b \times 1} \to \mathbb{R}^{c \times 1}$ are differentiable functions, then the Jacobian $J_f$ can be written by using gradients as $J_f(x) = \left(\nabla_x f^T\right)^T$. By chain rule $J_{g \circ f}(x) = J_g(f(x))J_f(x)$, we have the following chain rule for gradients

$$\nabla_x \left(g(f(x))^T\right) = \nabla_x \left(f(x)^T\right) \nabla_{f(x)} \left(g(f(x))^T\right). \tag{20}$$

We will also frequently use the following fact: For $\mathbf{A} \in \mathbb{R}^{a \times b}$,

$$\nabla_{\mathrm{vec}(\mathbf{A})} \mathrm{vec}(\mathbf{A})^T = \mathbf{I}_{ab}, \tag{21}$$

$$\nabla_{\mathrm{vec}(\mathbf{A})} \mathrm{vec}(\mathbf{A}^T)^T = \nabla_{\mathrm{vec}(\mathbf{A})} \mathrm{vec}(\mathbf{A})^T \mathbf{C}^{(b,a)} = \mathbf{C}^{(b,a)}. \tag{22}$$

For two matrices $\mathbf{A}, \mathbf{B}$ of the same size, we write $\mathbf{A} \preceq \mathbf{B}$ if $\mathbf{B} - \mathbf{A}$ is positive semi-definite. The partial ordering $\preceq$ is called the *Loewner ordering.*

# B    Details of model setup

## B.1    Details of angle-based approach

Consider the working model given by Equation (1) in the main text:

$$Y = \mu(\mathbf{X}) + \sum_{k=1}^{K} \delta_k(\mathbf{X})\mathbf{I}(A = k) + \epsilon.$$

To make the optimal ITR identifiable, we impose the constraint $\sum_{k=1}^{K} \delta_k(\mathbf{x}) = 0$ for all covariates $\mathbf{x}$. Under this model, $\delta_k(\mathbf{x})$ determines the optimal ITR for an individual with covariate $\mathbf{x}$, while $\mu(\mathbf{x})$ has no impact on the ITR. However, using $K$ functions with the sum-to-zero constraint can be computationally inefficient [55]. Instead, one can opt for simplex coding as an alternative, which inherently satisfies the sum-to-zero constraint.

In AD-Learning, proposed by [39], each of the $K$ treatments is represented as a vertex simplex on $\mathbb{R}^{K-1}$, denoted as $\mathbf{u}_k$, $k = 1, \ldots, K$. In particular, $\mathbf{u}_k$ is defined as

$$\mathbf{u}_k = \begin{cases} \frac{1}{\sqrt{K-1}}\mathbf{1}_{K-1} & \text{if } k = 1, \\ -\frac{1+\sqrt{K}}{\sqrt{(K-1)^3}}\mathbf{1}_{K-1} + \sqrt{\frac{K}{K-1}}\mathbf{e}_{k-1} & \text{if } k \geq 2, \end{cases}$$

where $\mathbf{1}_{K-1}$ is a $K-1$ dimensional vector with entries 1, and $\mathbf{e}_{k-1}$ is a $K-1$ dimensional vector with entries 0 except its $k$-th entry being 1. This vertex simplex has $K$ vertices with equal angles between them and an origin at the center of the simplex. All $\mathbf{u}_k$ have the same Euclidean norm 1. To estimate the optimal ITR, a decision function $\mathbf{f}(\mathbf{x}) = (f_1(\mathbf{x}), \ldots, f_{K-1}(\mathbf{x}))^T$ is constructed, assigning a $K-1$ dimensional vector to each covariate $\mathbf{x}$. Here, $\mathbf{f}(\cdot)$ can take a generic form, while we assume the linear decision function in the main text. The optimal ITR is reformulated by the authors as follows:

$$d^{\text{opt}}(\mathbf{x}) = \underset{k \in \{1,\ldots,K\}}{\arg\max} \ \mathbf{u}_k^T \mathbf{f}(\mathbf{x}) = \underset{k \in \{1,\ldots,K\}}{\arg\min} \ \angle(\mathbf{u}_k, \mathbf{f}(\mathbf{x})),$$

where $\angle(\cdot, \cdot)$ represents the angle between two vectors. Note that due to $\sum_{k=1}^{K} \mathbf{u}_k = \mathbf{0}$, $\sum_{k=1}^{K} \mathbf{u}_k^T \mathbf{f}(\mathbf{x}) = 0$ for any given function $\mathbf{f}$ and covariate value $\mathbf{x}$. Then we can use

$$\left(\sum_{k=1}^{K} \mathbf{u}_k \mathbf{I}(A = k)\right)^T \mathbf{f}(\mathbf{x})$$

instead of $\sum_{k=1}^{K} \delta_k(\mathbf{X})\mathbf{I}(A = k)$ to encode the treatment and covariate interaction. Specifically, we use $\mathbf{u}_A$ to denote $\left(\sum_{k=1}^{K} \mathbf{u}_k \mathbf{I}(A = k)\right)^T$ and assume $\mathbf{f}(\mathbf{x})$ to be linear in $\mathbf{x}$ (with the coefficient matrix $\mathbf{B}$). Therefore, terms pertaining to ITRs can be expressed as $\mathbf{u}_{a_i}^T \mathbf{B}^T \mathbf{x}_i$.

## B.2    Details of binary outcome

Again, under prespecified treatment rule class $\mathcal{D}$, the optimal ITR is given by

$$d^{\text{opt}}(\mathbf{x}) = \underset{k \in \{1,\ldots,K\}}{\arg\max} \ \mathbb{E}[Y|\mathbf{x}, A = k] = \underset{k \in \{1,\ldots,K\}}{\arg\max} \ \mathbf{u}_k^T \underbrace{\mathbb{E}\left[\frac{Y\mathbf{u}}{\pi(A, \mathbf{x})}\middle|\mathbf{x}\right]}_{=: \mathbf{f}_{\text{opt}}(\mathbf{x})}.$$

For the binary response setting, we assume the following logistic model

$$\mathbb{P}[Y = 1|\mathbf{x}, A = k] = \frac{\exp(\mathbf{u}_k^T \mathbf{f}_{\text{opt}}(\mathbf{x}))}{1 + \exp(\mathbf{u}_k^T \mathbf{f}_{\text{opt}}(\mathbf{x}))}.$$

Under this model, $\mathbf{f}_{\text{opt}}$ is indeed the optimal decision function under the angle-based direct learning framework since

$$
\begin{aligned}
d^{\text{opt}}(\mathbf{x}) &= \underset{k \in \{1,\dots,K\}}{\arg\max} \ \mathbb{P}[Y = 1 | \mathbf{x}, A = k] \\
&= \underset{k \in \{1,\dots,K\}}{\arg\max} \ \frac{\exp(\mathbf{u}_k^T \mathbf{f}_{\text{opt}}(\mathbf{x}))}{1 + \exp(\mathbf{u}_k^T \mathbf{f}_{\text{opt}}(\mathbf{x}))} \\
&= \underset{k \in \{1,\dots,K\}}{\arg\max} \ \mathbf{u}_k^T \mathbf{f}_{\text{opt}}(\mathbf{x}).
\end{aligned}
\tag{23}
$$

Furthermore, it can be checked that the function $\mathbf{f}_{\text{opt}}$ is *an* optimal solution to maximizing the IPW-weighted log-likelihood of the logistic regression model:

$$
\underset{\mathbf{f}}{\arg\min} \ \mathbb{E} \left[ -\frac{Y\mathbf{u}^T\mathbf{f}}{\pi(A, \mathbf{x})} + \frac{\log(\exp(\mathbf{u}^T\mathbf{f}) + 1)}{\pi(A, \mathbf{x})} \right],
\tag{24}
$$

under the assumption that the expectation and derivative w.r.t. $\mathbf{f}$ can be exchanged ([39]).

For the effective and robust ITR estimation, replacing IPW with DCBW leads to

$$
\underset{\mathbf{f}}{\arg\min} \ \mathbb{E} \left[ w(\mathbf{x}, A) \left\{ -Y\mathbf{u}^T\mathbf{f} + \log(\exp(\mathbf{u}^T\mathbf{f}) + 1) \right\} \right].
\tag{25}
$$

Now consider restricting it to the linear decision function class $\mathcal{F} = \{\mathbf{f}(\mathbf{X}) = \mathbf{B}^T\mathbf{X} : \mathbf{B} \in \mathbb{R}^{p \times (K-1)}\}$. Then the above (25) becomes

$$
\underset{\mathbf{B} \in \mathbb{R}^{p \times (K-1)}}{\arg\max} \ \mathbb{E} \left[ w(\mathbf{x}, A)\{Y\mathbf{u}^T\mathbf{B}^T\mathbf{x} - \log(\exp(\mathbf{u}^T\mathbf{B}^T\mathbf{x}) + 1)\} \right].
$$

Then, we can estimate the optimal parameter $\mathbf{B}$ by solving the following empirical loss minimization

$$
\underset{\mathbf{B} \in \mathbb{R}^{p \times (K-1)}}{\arg\min} \ \frac{1}{n} \sum_{i=1}^{n} w(\mathbf{x}_i, a_i) \left( -y_i \mathbf{u}_i^T \mathbf{B}^T \mathbf{x}_i + \log(\exp(\mathbf{u}^T \mathbf{B}^T \mathbf{x}_i) + 1) \right) \quad \text{s.t} \quad \|\mathbf{B}\|_1 \leq z, \tag{26}
$$

where we imposed additionally the $L_1$-penalization $\|\mathbf{B}\|_1 \leq z$, on $\mathbf{B}$ to promote a sparse solution. By the proposed algorithm 1, we can efficiently solve the problem (26).

# C   Details of proposed approaches

To keep the appendix as self-contained as possible, we provide descriptions of the building blocks of the proposed methods.

## C.1   Use of distributional covariate balancing weights

In this section, we provide the background about the use of DCBWs for constructing effective and robust ITRs. For additional details, please refer to the original papers.

Many existing ITR-Learning approaches such as AD-Learning and SABD-Learning use *inverse probability (of treatment) weighting* (IPW), which requires specifying a propensity score model for confounding control. However, propensity score methods have long been known to be highly sensitive to model misspecification, which yields biased estimates of causal effects [25, 30]. For this reason, methods that directly estimate weights based on balance-seeking objectives instead of using IPW have been proposed. These approaches include entropy balancing weights [20], stable balancing weights [61] as well as DCBWs [24, 27]. However, despite the weights serving a critical role in the estimation process, these advancements have mainly focused on improving *average treatment effect* (ATE) and are largely under-explored in the ITR literature. Significant improvements in the performance of ITRs are often left on the table when the weights are not treated with the same level of emphasis and focus as the other aspects of ITR learning.

To tackle these challenges, we propose the use of DCBWs, such as *energy balancing weights* (EBWs) [24] and *maximum mean discrepancy* (MMD) balancing weights [19, 27, 8], as an alternative to IPW for ITR-Learning. Unlike traditional approaches, these methods do not rely on any pre-specified functional forms for constructing weights. Instead, they focus on minimizing the distance of the weighted empirical distributions of covariates across treatment groups to reduce potential confounding. Even for data from randomized trials, they can help reduce finite sample covariate imbalance. The emphasis on finite sample balance and the model-independent nature of distributional weighting methods effectively mitigate biases due to model misspecification, resulting in more precise and reliable ITR estimation.

In the remainder of this section, we provide additional details about EBWs. The EBWs are derived from the concept of "weighted energy distance" which modifies the energy distance measure to allow for measurement of the distance between a weighted empirical distribution and a target empirical distribution. This measure captures the distance between a weighted *empirical cumulative distribution function* (ECDF) of covariates for a specific treatment group $a$, denoted as $\{\mathbf{X}_i\}_{i:A_i=a}$, and the combined ECDF of covariates, denoted as $\{\mathbf{X}_i\}_{i=1}^n$. Specifically, the weighted energy distance is defined as:

$$\mathcal{E}\left(F_{n,a,\mathbf{w}}, F_n\right) \equiv \frac{2}{n_a n} \sum_{i=1}^n \sum_{j=1}^n w_i \mathbf{I}\left(A_i = a\right) \left\|\mathbf{X}_i - \mathbf{X}_j\right\|_2 - \frac{1}{n_a^2} \sum_{i=1}^n \sum_{j=1}^n w_i w_j \mathbf{I}\left(A_i = A_j = a\right) \left\|\mathbf{X}_i - \mathbf{X}_j\right\|_2$$

$$- \frac{1}{n^2} \sum_{i=1}^n \sum_{j=1}^n \left\|\mathbf{X}_i - \mathbf{X}_j\right\|_2.$$

where $n_a = \sum_{i=1}^n \mathbf{I}(A_i = a)$. The weighted energy distance measures the distance between the weighted distribution of covariates among those with $A_i = a$ and the empirical distribution of covariates in the full sample; it takes value 0 if and only if these two distributions are identical.

The optimal weights are obtained by minimizing the sum of all the weighted energy distances of all treatment groups:

$$\mathbf{w}_n \in \operatorname*{argmin}_{\mathbf{w}=(w_1,\ldots,w_n)} \sum_{a \in \mathcal{A}} \mathcal{E}\left(F_{n,a,\mathbf{w}}, F_n\right) \text{ s.t. } \sum_{i=1}^n w_i I\left(A_i = a\right) = n_a \tag{27}$$

$$\text{for all } a \in \mathcal{A} \text{ and } w_i \geq 0 \text{ for } i = 1, \ldots, n.$$

Thus, the EBWs encourage the distributions of covariates for each treatment group to look like that of the target population, i.e. the full sample. Since (27) is a quadratic objective function along with linear constraints, it can be efficiently solved through interior point methods. Another advantage lies in the fact that it does not require parameter tuning. The estimated weights are effective in balancing

all covariates of the covariate distributions, not just limited to the first or second moments. Lastly, the data-driven characteristic of EBWs that do not rely on specific model assumptions contributes to more robust ITR estimates.

## C.2 Implementation of the PGD algorithm

We provide Algorithm 1 to solve the problem (6) using a *projected gradient descent* (PGD). The algorithm applies to either continuous or binary outcomes. For the projection onto the $L_1$-ball, we use the algorithm in [14].

---

**Algorithm 1** Projected Gradient Descent Algorithm to Estimate Decision Function for ITR-Learning

---

**Input:** Data set $(\mathbf{x}_i, a_i, y_i)_{i=1}^n \in \mathbb{R}^{n \times p} \times \{1, 2, \cdots, K\}^n \times \mathbb{R}^n$ ($\triangleright$ $y_i$ can be continuous or binary); loss function $L$

**Parameters**: $\mathbf{B} \in \mathbb{R}^{p \times (K-1)}$ (model coefficient), $z \geq 0$ ($L_1$-ball size); $\lambda_1 \geq 0$ (additional $L_2$-penalization for $R(\mathbf{B})$); $T \in \mathbb{N}$ (number of iterations), $\alpha_t \geq 0$ (stepsize).

**Constraints**: $L_1$-ball $\mathcal{B} := \{\mathbf{B} \in \mathbb{R}^{p \times (K-1)} \mid \|\mathbf{B}\|_1 \leq z\}$

   Compute $\mathbf{x}_i^* := \mathbf{x}_i \mathbf{u}_{a_i}^T$ for all $i = 1, \cdots, n$ for faster computation.

   Choose any point from $\mathbf{B}_0 \in \mathcal{B}$ as initialization.

   **For** $t = 1, 2, \ldots, T$ **do:**

   $$\mathbf{B}_t \leftarrow \Pi_{\mathcal{B}}\left(\mathbf{B}_{t-1} - \alpha_t \nabla_{\mathbf{B}} \mathcal{L}(\mathbf{B}_{t-1})\right).$$

   **End for**

**Output:** $\mathbf{B}_T$

---

To help the convenient implementation of the above PGD algorithm, we provide the gradient of the ITR objective below. For the continuous outcome, we have (see Proposition F.1)

$$\nabla_{\mathbf{B}} \mathcal{L}(\mathbf{B}) = \frac{1}{n} \sum_{i=1}^n \omega(a_i, \mathbf{x}_i) \left(\mathbf{x}_i \mathbf{x}_i^T \mathbf{B} \mathbf{u}_{a_i} \mathbf{u}_{a_i}^T - \frac{K}{K-1} y_i \mathbf{x}_i \mathbf{u}_{a_i}^T\right) + \lambda_2 \mathbf{B}. \tag{28}$$

For the binary outcome, we have (see Proposition F.2)

$$\nabla_{\mathbf{B}} \mathcal{L}(\mathbf{B}) = \frac{1}{n} \left(\sum_{i=1}^n \omega(a_i, \mathbf{x}_i) \left(\frac{\exp(\mathbf{u}_{a_i}^T \mathbf{B}^T \mathbf{x}_i)}{1 + \exp(\mathbf{u}_{a_i}^T \mathbf{B}^T \mathbf{x}_i)} - y_i\right) \mathbf{x}_i \mathbf{u}_{a_i}^T\right) + \lambda_2 \mathbf{B}. \tag{29}$$

## C.3 Additional improvements by dimension reduction: variable screening procedure

High dimensional covariates can significantly affect the performance of methods for estimating weights, outcome regression models, and the final ITR. Specifically, using screened variables is beneficial in estimating outcome regression models by focusing on fewer variables that significantly impact the outcome. Additionally, estimating balancing weights based on effect modifiers rather than all variables provides more reliable weighting estimates. Hence, the screening approach serves to reduce bias and enhance the efficiency of ITR-Learning.

Several techniques have been developed to choose appropriate variables in causal inference. One popular approach in the ITR setting is the group-lasso method introduced in [33] which filters out variables that only affect the treatment-free effect. For the ATE estimation, *sure independence screening* (SIS) method [16, 32] is widely used, which involves an initial screening step based on Pearson correlations between each covariate and the outcome. The covariates are then ranked based on these correlations, and only the top-ranked ones are used in the downstream analysis. While one could also use similar variable screening methods in the ITR setting, the use of Pearson correlation may limit their effectiveness in capturing nonlinear relationships, potentially leading to biased estimates of the ITR.

To address this issue, alternative approaches that employ non-linear dependence tests are explored, including the ball covariance, mutual information test, and distance covariance [37, 42, 48]. Ball covariance, previously applied to variable selection in ATE estimation, has been found to be effective [49]. Distance covariance is another way to capture the nonlinear dependence between two random

vectors/variables [48]. The test compares the observed distances between pairs of observations to the expected distances under the assumption of independence. A zero value of distance covariance indicates independence between two random vectors. If the observed distances are significantly far from the expected distances, it indicates statistical dependence between the random vectors. The $p$-value is calculated by comparing the observed empirical distance with the distribution of the same test statistic obtained from random permutations of the data under the assumption of independence. We propose adapting the distance covariance due to its computational efficiency and empirical performance for the settings we explored.

Now, we introduce our proposed variable screening procedure. In a setting with $K$ treatments, we aim to find which covariates have conditional dependence on the outcome given the treatment. Doing so allows the identification of covariates that either modify the treatment effect or otherwise affect the outcome (i.e. are precision variables). To do so, for each covariate $X_j$ where $j \in \{1, \ldots, p\}$, conduct the distance covariance test between covariate and outcome $K$ times, conditioning on each treatment value. This conditioning on the treatment creates subgroups based on treatment assignments, allowing the assessment of the dependence between the covariate and the outcome within each subgroup. Then we set the $p$-value of the covariate $X_j$ as the minimum p-value from these $K$ tests. This approach ensures that we retain as many relevant variables as possible during this preliminary variable screening stage. For the same reason, we avoid the use of multiple comparison adjustments such as Bonferroni correction. Instead, we opt to select variables with a significance level of 5%. We perform the same steps for all covariates in the dataset. These procedures are summarized in Algorithm 2.

---

**Algorithm 2** Variable Screening using Distance Covariance

---

**Input:** $\mathbf{X} \in \mathbf{R}^{n \times p}$; $Y \in \mathbf{R}^n$; $A \in \{1, \ldots, K\}$.
Initialize $\hat{\mathbf{X}} = \{\}$.
**for** $j = 1, 2, \ldots, p$ **do**
    **for** $k = 1, 2, \ldots, K$ **do**
        Perform `distance covariance test`$(X_j, Y|A = k)$ and obtain $p$-value, $p_{j,k}$.
    **end for**
    $p_j \leftarrow \min(p_{j,1}, p_{j,2}, \ldots, p_{j,K})$.
    **if** $p_j < 0.05$ **then**
        $\hat{\mathbf{X}} \leftarrow \hat{\mathbf{X}} \cup \{\mathbf{X}_j\}$.
    **end if**
**end for**
**Output:** $\check{\mathbf{X}}$, the set of screened variables containing effect modifiers and precision variables.

---

This conditional independence test is crucial to retain effect modifiers and precision variables while filtering out instrumental variables associated with treatment but not the outcome [49]. While acknowledging that the procedure is ad-hoc, the simulation results show that it effectively reduces the number of covariates to estimate covariate balancing weights and outcome regression models, which improves efficiency and helps avoid overfitting.

### C.4  Additional improvements by variance reduction: combining inverse variance weighting with augmented outcome

Our additional recommendation includes dually using both augmented outcomes and inverse variance weighting for variance reduction. Under our working model (1), the outcome can be explained by the sum of the treatment-free effects of covariates, the interaction effect between covariates and each treatment, and random noise. As the interaction effect alone determines optimal treatment assignments, the treatment-free effect acts as extra noise. This noise can be mitigated using outcome model augmentation. One way to implement outcome augmentation is to use the augmented outcomes $\check{Y} := Y - \hat{\mu}(\mathbf{X})$ in place of the original outcomes for continuous outcomes when applying an ITR estimation method. Flexible techniques such as random forest and neural networks can be utilized to estimate the treatment-free effect. This is the first implementation for multi-category treatments with DCBWs, whereas previous work focused on binary treatments with IPW [44].

Augmented outcomes can refine ITR estimates when correctly specified. Yet, misspecification of either outcome or treatment-free effect models can lead to heteroscedastic errors due to residual treatment-free effects. This is where SABD-Learning's inverse variance weighting helps, providing

robustness against misspecification. SABD-Learning extends AD-Learning designed for homoscedasticity [39] to account for heteroscedasticity, i.e. when the conditional variance of the (potential) outcome given the covariates is not constant [44]. This method estimates the conditional variance of outcomes and reweights the final objective function by adjusting each sample's contribution based on the inverse of the estimated variance. Therefore, we propose to combine outcome augmentation and inverse variance weighting for robust and efficient ITR learning, as they are complementary methods.

The key idea behind outcome augmentation is to add an asymptotically unbiased quantity to the objective function (negative log-likelihood function) so that the new optimizer for the 'augmented objective function' has a provably smaller variance than the original estimator. To do this, let $h$ be some unknown (to be determined) augmentation function depending on the covariate. The key property of asymptotic unbiasedness is

$$\frac{1}{n}\sum_{i=1}^{n} w(a_i, \mathbf{x}_i)\mathbf{u}_{a_i}^T \mathbf{B}^T h(\mathbf{x}_i) \to 0 \quad \text{as } n \to \infty \text{ in probability for all } \mathbf{B} \in \mathbb{R}^{p\times(K-1)}. \tag{30}$$

We will justify this property in Corollary C.3 toward the end of this section. Given (30), we have

$$\ell(\mathbf{B}) \approx \ell(\mathbf{B}) + \frac{1}{n}\sum_{i=1}^{n} w_i \mathbf{u}_{a_i}^T \mathbf{B}^T h(\mathbf{x}_i), \tag{31}$$

where $\approx$ here means asymptotic equality in probability as $n \to \infty$. Denote the 'augmented objective function' on the right-hand side as $\ell_{aug}(\mathbf{B})$.

In the remainder of this section, we introduce the optimal augmented function $h_{\text{opt}}(\mathbf{X})$ that minimizes the variance of the estimator, while maintaining asymptotic unbiasedness. We demonstrate that outcome augmentation remains valid when IPW is replaced with DCBWs, such as EBWs and MMD balancing weights. Our approach extends the work of [50, 9] to a more general setting, incorporating other types of covariate balancing weights that have asymptotic consistency (38) in a multi-category treatment setting.

Let $S(\mathbf{B})$ be the derivative of the original objective function $\ell(\mathbf{B})$ with respect to $\mathbf{B}$. Then setting the optimal $h(\mathbf{X})$ is equivalent to minimizing the conditional variance of

$$\widehat{\mathbf{B}}_a - \mathbf{B}^\star \approx \underbrace{S(\mathbf{B}) - w(A, \mathbf{X})h(\mathbf{X})\mathbf{u}_A^T | \mathbf{X} = \mathbf{x}}_{=:\mathbf{Z}}$$

where $\widehat{\mathbf{B}}_a$ is the minimizer of $\ell_{aug}(\mathbf{B})$, and $\mathbf{B}^\star$ is the minimizer of $\mathbb{E}[\ell(\mathbf{B})]$. Using the property of asymptotic unbiasedness Theorem C.2, which is $\mathbb{E}[\mathbf{Z}] = 0$, we need to minimize $\mathbb{E}[\|\mathbf{Z}\|_F^2]$ with respect to $h$ to minimize the conditional variance of $\mathbf{Z}$. Using the property of matrix trace,

$$\nabla_h \mathbb{E}[\|\mathbf{Z}\|_F^2] = \nabla_h \mathbb{E}[\text{tr}(\mathbf{Z}^T\mathbf{Z})] = -2\mathbb{E}[w(A, \mathbf{X})\,\mathbf{Z}\,\mathbf{u}_A)|\mathbf{X} = \mathbf{x}],$$

and setting $\nabla_h \mathbb{E}[\|\mathbf{Z}\|_F^2] = 0$. Thus, $h_{\text{opt}}(\mathbf{X})$ satisfies

$$\mathbb{E}[w(A, \mathbf{X})(S(\mathbf{B}) - w(A, \mathbf{X})h_{\text{opt}}(\mathbf{X})\mathbf{u}_A^T)\mathbf{u}_A)|\mathbf{X} = \mathbf{x}] = 0.$$

Since $\mathbf{u}_A^T\mathbf{u}_A = 1$, we have

$$h_{\text{opt}}(\mathbf{x}) = \frac{\mathbb{E}[w(A, \mathbf{X})S(\mathbf{B}^\star)\mathbf{u}_A|\mathbf{X} = \mathbf{x}]}{\mathbb{E}[w(A, \mathbf{X})^2|\mathbf{X} = \mathbf{x}]}. \tag{32}$$

Now, we determine the functional form of $h_{\text{opt}}(\mathbf{X})$ for both continuous and binary outcomes. For simplicity, let us define

$$w := w(A, \mathbf{X}), \quad t_i := \mathbf{u}_{a_i}^T \mathbf{B}^T \mathbf{x}_i$$
$$R_1(\mathbf{x}) := \mathbb{E}[w^2\mathbf{u}_A|\mathbf{X} = \mathbf{x}], \quad R_2(\mathbf{x}) := \mathbb{E}[w^2Y|\mathbf{X} = \mathbf{x}], \quad c_1 = E[w^2|\mathbf{X} = \mathbf{x}].$$

Then for continuous outcomes, using (28) without regularization term for $S(\mathbf{B})$ and (32), we have

$$\ell_{aug}(\mathbf{B})$$

$$= \frac{1}{n}\sum_{i=1}^{n}\left[w_i\left(\frac{K}{K-1}y_i - \mathbf{u}_{a_i}^T\mathbf{B}^T\mathbf{x}_i\right)^2 - w_i\mathbf{u}_{a_i}^T\mathbf{B}^T h_{\text{opt}}(\mathbf{x}_i)\right]$$

$$= \frac{1}{n}\sum_{i=1}^{n}\left[w_i\left(\frac{K}{K-1}y_i - \mathbf{u}_{a_i}^T\mathbf{B}^T\mathbf{x}_i\right)^2 - w_i\mathbf{u}_{a_i}^T\mathbf{B}^T\frac{1}{\widehat{c}_1}\left(\mathbf{x}_i\mathbf{x}_i^T\mathbf{B}\widehat{R}_1(\mathbf{x}_i) - \frac{2K}{K-1}\mathbf{x}_i\widehat{R}_2(\mathbf{x}_i)\right)\right]$$

$$= \frac{1}{n}\sum_{i=1}^{n}w_i\left[\left(\frac{K}{K-1}y_i - t_i\right)^2 - t_i\left(\mathbf{x}_i^T\mathbf{B}\frac{1}{\widehat{c}_1}\widehat{R}_1(\mathbf{x}_i)\right) + t_i\left(\frac{2K}{K-1}\frac{1}{\widehat{c}_1}\widehat{R}_2(\mathbf{x}_i)\right)\right]$$

$$= \frac{1}{n}\sum_{i=1}^{n}w_i\left[\left(t_i - \frac{K}{K-1}\left(y_i - \frac{1}{\widehat{c}_1}\widehat{R}_2(\mathbf{x}_i)\right)\right)^2 + \text{constant w.r.t }\mathbf{B}\right],$$

assuming $\widehat{R}_1(\mathbf{x}_i)/\widehat{c}_1 = 0$. Therefore, we obtain the optimal augmented function with the minimum estimation variance by properly defining $R_2(\mathbf{X})$. Our simulation studies demonstrate that setting $R_2(\mathbf{X}) = \mathbb{E}[Y|\mathbf{X}=\mathbf{x}]$ leads to significant improvement.

Similarly, for binary outcomes, let us define

$$Q_1(\mathbf{x}) := \mathbb{E}\left[w^2\frac{\exp(\mathbf{u}_A^T\mathbf{B}^T\mathbf{X})}{1+\exp(\mathbf{u}_A^T\mathbf{B}^T\mathbf{X})}\middle|\mathbf{X}=\mathbf{x}\right], \qquad Q_2(\mathbf{x}) := \mathbb{E}[w^2 Y|\mathbf{X}=\mathbf{x}].$$

Then using (29) without regularization term for $S(\mathbf{B})$ and (32), we have

$$\ell_{\text{aug}}(\mathbf{B}) = \frac{1}{n}\sum_{i=1}^{n}w_i\left[-y_i\mathbf{u}_{a_i}^T\mathbf{B}^T\mathbf{x}_i + \log(1+\exp(\mathbf{u}_{a_i}^T\mathbf{B}^T\mathbf{x}_i)) + \mathbf{u}_{a_i}^T\mathbf{B}^T h_{\text{opt}}(\mathbf{x}_i)\right]$$

$$= \frac{1}{n}\sum_{i=1}^{n}w_i\left[-y_i t_i + \log(1+\exp(t_i)) + \frac{1}{\widehat{c}_1}t_i(\widehat{Q}_1(\mathbf{x}) - \widehat{Q}_2(\mathbf{x}))\right]$$

$$= \frac{1}{n}\sum_{i=1}^{n}w_i\left[-t_i\left(y_i - \frac{1}{\widehat{c}_1}(\widehat{Q}_1(\mathbf{x}) - \widehat{Q}_2(\mathbf{x}))\right) + \log(1+\exp(t_i))\right].$$

Therefore, by selecting proper $Q_1(\mathbf{X})$ and $Q_2(\mathbf{X})$, the minimizer of $\ell_{\text{aug}}(\mathbf{B})$ has smaller variance than that of the original estimator.

Now we prove asymptotic unbiasedness of $\frac{1}{n}\sum_{i=1}^{n}w(a_i,\mathbf{x}_i)\mathbf{u}_{a_i}^T\mathbf{B}^T h(\mathbf{x}_i)$. First, we consider IPW-based conditions for multi-category treatments.

**Lemma C.1.** *Let $\mathcal{A} = \{1,\ldots,K\}$ be the possible treatment sets. Assume we have i.i.d. observations $(\mathbf{x}_i, a_i)$ from a population distribution of $(\mathbf{X}, A)$ such that $\mathbb{P}(A = a \mid \mathbf{x})$ is uniformly positive for all $a, \mathbf{x}$. Define*

$$T := \frac{1}{n}\sum_{i=1}^{n}\frac{1}{\mathbb{P}(A=a_i|\mathbf{x}_i)}h(\mathbf{x}_i)g(a_i). \tag{33}$$

*If $g$ is any function with $\sum_{j\in\mathcal{A}}g(j) = 0$, then for any function $h(\mathbf{X}) : \mathbb{R}^p \to \mathbb{R}^p$, then $T \to 0$ as $n \to \infty$ in probability.*

*Proof.* First note that by iterated expectation,

$$\mathbb{E}[T] = \mathbb{E}\left[\frac{h(\mathbf{X})g(A)}{\mathbb{P}(A|\mathbf{X})}\right] = \mathbb{E}_{\mathbf{X}}\left[\mathbb{E}_A\left[\frac{h(\mathbf{X})g(A)}{\mathbb{P}(A|\mathbf{X})}\middle|\mathbf{X}\right]\right] \tag{34}$$

$$= \mathbb{E}_{\mathbf{X}}\left[\sum_{j\in\mathcal{A}}\frac{h(\mathbf{X})g(j)}{\mathbb{P}(A=j|\mathbf{X})}\mathbb{P}(A=j|\mathbf{X})\right] \tag{35}$$

$$= \mathbb{E}_{\mathbf{X}}\left[h(\mathbf{X})\sum_{j\in\mathcal{A}}g(j)\right] \tag{36}$$

$$= 0. \tag{37}$$

Then by the standard weak law of large numbers for i.i.d. samples, the empirical mean $T$ converges in probability to its population mean $\mathbb{E}[T] = 0$ as $n \to \infty$. $\qquad\square$

Note that in the statement of the Lemma C.1, the choice of such $g(j)$ can be each treatment vertex under the AD-Learning [39]. Now we replace IPW $\mathbb{P}(A = a_i | \mathbf{x}_i)$ with covariate balancing weights $w(a_i, \mathbf{x}_i)$ that has the following asymptotic consistency property:

$$\lim_{n \to \infty} \frac{1}{n} \sum_{i=1}^{n} \left( w(a_i, \mathbf{x}_i) - \frac{1}{\mathbb{P}(a_i | \mathbf{x}_i)} \right)^2 = 0. \tag{38}$$

For instance, *maximum mean discrepancy* (MMD) balancing weights, a type of distributional covariate energy balancing weight, satisfy the property (38) [8].

**Theorem C.2** (Asymptotic unbiasedness of augmented function). *Assume we have i.i.d. observations $(\mathbf{x}_i, a_i)$ from population distribution of $(\mathbf{X}, A)$. If $g : \mathcal{A} \to \mathbb{R}$ is any bounded function with $\sum_{j \in \mathcal{A}} g(j) = 0$, then for any bounded function $h(\mathbf{x}) : \mathbb{R}^p \to \mathbb{R}^p$, define*

$$T_0 := \frac{1}{n} \sum_{i=1}^{n} w(a_i, \mathbf{x}_i) h(\mathbf{x}_i) g(a_i), \tag{39}$$

*where the weights $w(a_i, \mathbf{x}_i)$ satisfy (38). Then $T_0 \to 0$ as $n \to \infty$ in probability. That is, $T_0$ is an asymptotically mean zero, making it suitable to be used as the augmented function.*

*Proof.* By the weak law of large numbers for i.i.d. samples, the empirical mean $T_0$ converges in probability to its population mean $\mathbb{E}[T_0] = \mathbb{E}[w(A, \mathbf{X}) h(\mathbf{X}) g(A)]$. By continuous mapping theorem, $\|T_0\|^2$ converges to $\|\mathbb{E}[T_0]\|^2$ as $n \to \infty$ in probability. Then we have

$$\|\mathbb{E}[T_0]\|^2 = \|\mathbb{E}[w(A, \mathbf{X}) h(\mathbf{X}) g(A)]\|^2 \overset{(a)}{=} \left\| \mathbb{E}[w(A, \mathbf{X}) h(\mathbf{X}) g(A)] - \mathbb{E}\left[ \frac{h(\mathbf{X}) g(A)}{\mathbb{P}(A | \mathbf{X})} \right] \right\|^2$$

$$= \left\| \mathbb{E}\left[ \left( w(A, \mathbf{X}) - \frac{1}{\mathbb{P}(A | \mathbf{X})} \right) h(\mathbf{X}) g(A) \right] \right\|^2$$

$$\overset{(b)}{\leq} \mathbb{E}\left[ \left\| \left( w(A, \mathbf{X}) - \frac{1}{\mathbb{P}(A | \mathbf{X})} \right) h(\mathbf{X}) g(A) \right\|^2 \right]$$

$$\leq C \, \mathbb{E}\left[ \left( w(A, \mathbf{X}) - \frac{1}{\mathbb{P}(A | \mathbf{X})} \right)^2 \right]$$

$$\overset{(c)}{=} \lim_{n \to \infty} C \frac{1}{n} \sum_{i=1}^{n} \left( w(a_i, \mathbf{x}_i) - \frac{1}{\mathbb{P}(a_i | \mathbf{x}_i)} \right)^2 \to 0,$$

where $C$ is any constant such that $\|h(\mathbf{X}) g(A)\|^2 \leq C$. For (a), we use Lemma C.1; for (b) and (c), we use Jensen's inequality and the law of large numbers, respectively. The last term converges to 0 due to the property (38). Lastly, since $\|T_0\|^2$ converges to 0 in probability, we have $T_0$ converges to 0 in probability by continuous mapping theorem. $\qquad\square$

The following corollary is useful in deriving outcome augmentation under the AD-Learning framework.

**Corollary C.3.** *Assume the AD-learning setting. For any function $h(\mathbf{X})$ depending only on the covariate $\mathbf{X}$, and for each parameter matrix $\mathbf{B} \in \mathbb{R}^{p \times (K-1)}$, denoting,*

$$H_0(\mathbf{B}) := w(A, \mathbf{X}) \mathbf{u}_A^T \mathbf{B}^T h(\mathbf{X}),$$

*we have $\mathbb{E}[H_0(\mathbf{B})] = 0$. Furthermore, assuming the weights $w(A, \mathbf{X})$ satisfying the consistency property (38), in probability as $n \to \infty$,*

$$\frac{1}{n} \sum_{i=1}^{n} w(a_i, \mathbf{x}_i) \mathbf{u}_{a_i}^T \mathbf{B}^T h(\mathbf{x}_i) \to 0. \tag{40}$$

*Proof.* Follows immediately from Theorem C.2 by noting that $\sum_{a=1}^{K} \mathbf{u}_a = 0$. $\qquad\square$

## D  Spectral characterization of optimal covariate balancing weight

Recall the definition of the weighted design matrix:

$$\boldsymbol{\Psi} := \frac{1}{n} \sum_{i=1}^{n} w_i (\mathbf{u}_{a_i} \mathbf{u}_{a_i}^T) \otimes \left( \mathbf{x}_i \mathbf{x}_i^T \right). \tag{41}$$

This is the $p(K-1) \times p(K-1)$ design matrix encoding the sample of patients with covariate $\mathbf{x}_i$ and observed treatment $a_i$, for $i = 1, \ldots, n$. We are free to choose the weight $w_i = \omega(a_i, \mathbf{x}_i)$ for the $i$th patient as a function as long as they sum to one. What is the best way to choose such weights? Our main results (Theorems 3.3, 3.5) suggest that it is best to choose them in a way that the minimum eigenvalue of $\boldsymbol{\Psi}$ is as large as possible.

To illustrate this point, suppose for simplicity the covariates are categorical and assume values in a finite set $\mathcal{X}$. We decompose the summation in (41) according to the treatment and covariate as follows:

$$\boldsymbol{\Psi} = \frac{1}{n} \sum_{(a, \mathbf{x}) \in \mathcal{A} \times \mathcal{X}} w(a, \mathbf{x}) \underbrace{(\mathbf{u}_{a_i} \mathbf{u}_{a_i}^T) \otimes \left( \mathbf{x}_i \mathbf{x}_i^T \right)}_{=:\boldsymbol{\Sigma}(a, \mathbf{x})} \underbrace{\sum_{i=1}^{n} \mathbf{1}(a_i = a, \, \mathbf{x}_i = \mathbf{x})}_{=:n(a, \mathbf{x})} \tag{42}$$

$$= \sum_{(a, \mathbf{x}) \in \mathcal{A} \times \mathcal{X}} w(a, \mathbf{x}) \frac{n(a, \mathbf{x})}{n} \boldsymbol{\Sigma}(a, \mathbf{x}). \tag{43}$$

Note that $n(a, \mathbf{x})$ is the number of patients with covariate $\mathbf{x}$ and treatment $a$, while $\boldsymbol{\Sigma}(a, \mathbf{x})$ is the $p(K-1) \times p(K-1)$ covariance matrix representing the treatment-covariate pair $(a, \mathbf{x})$.

1. Under IPW $w(a, \mathbf{x}) = \frac{1}{\pi(a|\mathbf{x})}$, since $n(a, \mathbf{x}) \sim \text{Binom}(n, \pi(a|\mathbf{x}))$ (under suitable assumption), using a law of large numbers and central limit theorem, we have

   $$w(a, \mathbf{x}) \frac{n(a, \mathbf{x})}{n} = 1 + O_{\mathbb{P}}(n^{-1/2}).$$

   Consequently, the covariance matrix $\boldsymbol{\Sigma}(a, \mathbf{x})$ of every treatment-covariate pair $(a, \mathbf{x})$ contributes nearly equally to the weighted design matrix $\boldsymbol{\Psi}$.

   If distinct covariates are encoded as one-hot encoding vectors, then IPW maximizes the minimum eigenvalue of $\boldsymbol{\Psi}$. To see this, for simplicity, assume we have binary treatments ($K = 2$ and $u_1 = 1$, $u_2 = -1$) and two discrete covariates (e.g., 'Male' and 'Female') encoded as one-hot vectors $(1, 0)^T$ and $(0, 1)^T$. Then we can write $\boldsymbol{\Psi}$ as

   $$n\boldsymbol{\Psi} = w(1, M)n(1, M) \begin{bmatrix} 1 & 0 \\ 0 & 0 \end{bmatrix} + w(2, M)n(2, M) \begin{bmatrix} 1 & 0 \\ 0 & 0 \end{bmatrix}$$

   $$+ w(1, F)n(1, F) \begin{bmatrix} 0 & 0 \\ 0 & 1 \end{bmatrix} + w(2, F)n(2, F) \begin{bmatrix} 0 & 0 \\ 0 & 1 \end{bmatrix}$$

   $$= \begin{bmatrix} w(1, M)n(1, M) + w(2, M)n(2, M) & 0 \\ 0 & w(1, F)n(1, F) + w(2, F)n(2, F) \end{bmatrix}.$$

   Notice that the minimum eigenvalue of the last $2 \times 2$ matrix is the minimum of its two diagonal entries. With IPW, each weighted count $w(a, x)n(a, x)$ becomes a constant, and $\boldsymbol{\Psi}$ above becomes the identity matrix, achieving the largest possible minimum eigenvalue. Also, in this case, $\boldsymbol{\Psi}$ has condition number 1 so the ITR optimization landscape becomes well-conditioned.

2. Using the fact that the minimum eigenvalue is a positive homogeneous and concave function on the space of symmetric matrices, we have

   $$\lambda_{\min}(\boldsymbol{\Psi}) \geq \sum_{a \in \mathcal{A}} \lambda_{\min} \left( \underbrace{\sum_{\mathbf{x} \in \mathcal{X}} w(a, \mathbf{x}) \frac{n(a, \mathbf{x})}{n} \boldsymbol{\Sigma}(a, \mathbf{x})}_{=:\boldsymbol{\Sigma}_w(a)} \right).$$

Here $\boldsymbol{\Sigma}_w(a)$ is the aggregated weighted covariance matrix for sub-population with treatment $a$. In the above derivation, we used that $\boldsymbol{\Psi} = \sum_a \boldsymbol{\Sigma}_w(a)$ by (42) and definition and the following computation using Jensen's inequality:

$$
\begin{aligned}
\lambda_{\min}\left(\sum_a \boldsymbol{\Sigma}_w(a)\right) &= |\mathcal{A}|\lambda_{\min}\left(\sum_a \frac{1}{|\mathcal{A}|}\boldsymbol{\Sigma}_w(a)\right) \\
&\geq |\mathcal{A}|\sum_a \frac{1}{|\mathcal{A}|}\lambda_{\min}(\boldsymbol{\Sigma}_w(a)) = \sum_a \lambda_{\min}(\boldsymbol{\Sigma}_w(a)).
\end{aligned}
$$

If we choose the weights so that these matrices are identical, then each treatment group contributes equally to the lower bound above for $\lambda_{\min}(\boldsymbol{\Psi})$. This condition is satisfied, for example, when $w$ is distributional covariate balancing weights, which ensure that the covariate distributions across all treatment groups are identical. There is a remaining degree of freedom for specifying the common covariate distribution, denoted by $\mu$, for the treatment groups. To maximize the lower bound for $\lambda_{\min}(\boldsymbol{\Psi})$, $\mu$ should be chosen to maximize $\lambda_{\min}(\boldsymbol{\Sigma}_w(\cdot))$. The specific solution to this optimization depends on the structure of $\boldsymbol{\Psi}$. In three-way balancing methods [8], $\mu$ is typically chosen to be close to the target population distribution. Another common approach, as discussed earlier, is IPW, which sets $\mu$ as the uniform distribution over the covariate space $\mathcal{X}$.

# E   Convergence analysis of PGD for ITR estimation

In this section, we analyze the convergence properties of Algorithm 1 for ITR-Learning problem (6). Below we state the full version of Theorem 3.3 in the main text.

**Theorem E.1** (Convergence rate of PGD for ITR-Learning). *Let $(\mathbf{B}_t)_{t \geq 0}$ denote the sequence of parameters obtained by the PGD algorithm* (7) *for ITR-Learning problem* (6) *with arbitrary initialization $\mathbf{B}_0$. Suppose Assumptions 3.1 and 3.2 hold. Let $\mathbf{B}^\star$ denote the unique global optimum of* (6). *Then the following hold:*

**(i)** *(Optimization landscape) The ITR objective $\mathcal{L}(\mathbf{B})$ in* (6)*, is $\mu$-strongly convex and $L$-smooth, where $\mu, L$ are as in* (10).

**(ii)** *(Linear convergence with fixed stepsize) Assume constant stepsize $\alpha_t \equiv \alpha < 2/L$. Denote the contraction constant $\rho(\alpha) := \max\{|1 - \alpha L|, |1 - \alpha \mu|\} \in (0, 1)$. Then for all $t \geq 1$,*

$$\|\mathbf{B}_t - \mathbf{B}^\star\|_F^2 \leq 4\rho(\alpha)^t \lambda_1^2. \tag{44}$$

*In particular, if $\alpha = \frac{2}{\mu + L}$, then $\rho(\alpha) = \frac{L - \mu}{L + \mu} \in [0, 1)$.*

**(iii)** *(Sublinear convergence with diminishing stepsize) Assume diminishing stepsize $\alpha_t = \frac{\beta}{\gamma + t}$, where $\rho := \frac{2\mu L}{\mu + L}$ and $\beta > 1/\rho$, $\gamma > 0$ are constants. Then for all $t \geq 1$,*

$$\|\mathbf{B}_t - \mathbf{B}^\star\|_F^2 \leq \frac{\nu}{\gamma + t}, \tag{45}$$

*where $\nu := \max\left\{ \frac{2\beta^2 \|\nabla f(\mathbf{B}^\star)\|_F^2}{\beta\rho - 1}, 4(\gamma + 1)\lambda_1^2 \right\}$. In particular, the assertion holds for $\beta = 2/\rho = \frac{\mu + L}{\mu L}$ and $\nu = \max\left\{ \frac{2(\mu + L)^2 \|\nabla f(\mathbf{B}^\star)\|_F^2}{\mu^2 L^2}, 4(\gamma + 1)\lambda_1^2 \right\}$.*

The proof of Theorem E.1 relies on two ingredients: a general result on the convergence of PGD (Lemma E.2) and a local landscape analysis of the ITR objective (Lemma E.3).

We first establish a general result on the convergence rate of PGD to find the minimizer of a strongly convex and smooth objective $f$ within a convex constraint $\mathcal{B}$. Our argument is mostly based on the standard convergence analysis of PGD in the optimization literature [5, 4]. A new element we add is that for analyzing constrained optimization problems, the gradient of the objective $f$ at the minimizer $\mathbf{B}^\star$ does not need to be zero, which only needs to be in the negative normal cone at the constraint set $\mathcal{B}$. In our analysis, we treat the possibly nonzero gradient norm $\|\nabla f(\mathbf{B}^\star)\|_F$ as 'noise' (variance) of a stochastic gradient oracle in standard stochastic gradient descent analysis [4].

**Lemma E.2** (Convergence rate of PGD for strongly convex and smooth objectives). *Let $f : \mathbb{R}^{p \times (K-1)} \to \mathbb{R}$ be a $L$-smooth and $\mu$-strongly convex function for some $\mu, L > 0$. Suppose $f^\star := \inf_{\mathbf{B} \in \mathcal{B}} f(\mathbf{B}) > -\infty$ and denote $\mathbf{B}^\star := \arg\min_{\mathbf{B} \in \mathcal{B}} f(\mathbf{B})$, where $\mathcal{B}$ is a convex constraint set of $\mathbb{R}^{p \times (K-1)}$. Consider the following PGD iterates:*

$$\mathbf{B}_{t+1} \leftarrow \Pi_{\mathcal{B}} \left\{ \mathbf{B}_t - \alpha_t \nabla_{\mathbf{B}} f(\mathbf{B}_t) \right\}. \tag{46}$$

*Then the following hold:*

**(i)** *(Linear convergence with fixed stepsize) Suppose $f$ is twice differentiable and $\alpha_t \equiv \alpha < 2/L$. Let contraction constant $\rho(\alpha) := \max\{|1 - \alpha L|, |1 - \alpha \mu|\} \in (0, 1)$. Then for all $t \geq 1$,*

$$\|\mathbf{B}_t - \mathbf{B}^\star\|_F^2 \leq \rho(\alpha)_t \|\mathbf{B}_0 - \mathbf{B}^*\|_F^2. \tag{47}$$

*The bound $\rho(\alpha)$ on the linear convergence rate is minimized at $\alpha = \frac{2}{\mu + L}$ with minimum value $\frac{L - \mu}{L + \mu}$.*

**(ii)** *(Sublinear convergence with diminishing stepsize) Denote $\rho := \frac{2\mu L}{\mu + L}$. Suppose $\alpha_t = \frac{\beta}{\gamma + t}$, where $\beta > 1/\rho$, $\gamma > 0$ are constants. Then for all $t \geq 1$,*

$$\|\mathbf{B}_t - \mathbf{B}^\star\|_F^2 \leq \frac{\nu}{\gamma + t}, \tag{48}$$

*where $\nu := \max\left\{ \frac{2\beta^2 \|\nabla f(\mathbf{B}^\star)\|_F^2}{\beta\rho - 1}, (\gamma + 1)\|\mathbf{B}_1 - \mathbf{B}^\star\|_F^2 \right\}$. In particular, the assertion holds for $\beta = 2/\rho = \frac{\mu + L}{\mu L}$ and $\nu = \max\left\{ \frac{2(\mu + L)^2 \|\nabla f(\mathbf{B}^\star)\|_F^2}{\mu^2 L^2}, (\gamma + 1)\|\mathbf{B}_1 - \mathbf{B}^\star\|_F^2 \right\}$.*

*Proof.* Let $\bar{\mathbf{B}}_t := \mathbf{B}_t - \alpha_t \nabla_{\mathbf{B}} f(\mathbf{B}_t)$. Then

$$\|\bar{\mathbf{B}}_t - \mathbf{B}^*\|_F^2 = \|\bar{\mathbf{B}}_t - \mathbf{B}_t\|_F^2 + \|\mathbf{B}_t - \mathbf{B}^*\|_F^2 + 2\langle \bar{\mathbf{B}}_t - \mathbf{B}_t, \mathbf{B}_t - \mathbf{B}^* \rangle$$

$$= \alpha_t^2 \|\nabla_{\mathbf{B}} f(\mathbf{B}_t)\|_F^2 + \|\mathbf{B}_t - \mathbf{B}^*\|_F^2 - 2\alpha_t \langle \mathbf{B}_t - \mathbf{B}^*, \nabla_{\mathbf{B}} f(\mathbf{B}_t) \rangle$$

$$\leq \alpha_t^2 \|\nabla_{\mathbf{B}} f(\mathbf{B}_t)\|_F^2 + \|\mathbf{B}_t - \mathbf{B}^*\|_F^2 - 2\alpha_t \langle \mathbf{B}_t - \mathbf{B}^*, \nabla_{\mathbf{B}} f(\mathbf{B}_t) - \nabla_{\mathbf{B}} f(\mathbf{B}^\star) \rangle$$

using that $\langle \nabla f(\mathbf{B}^\star), \mathbf{B}_t - \mathbf{B}^\star \rangle \geq 0$ since $\mathbf{B}^\star$ is a stationary point of $f$ over $\mathcal{B}$. Using the co-coercivity of $\mu$-strongly convex and $L$-smooth functions and, we have

$$\|\bar{\mathbf{B}}_t - \mathbf{B}^*\|_F^2 \leq \alpha_t^2 \|\nabla f(\mathbf{B}_t)\|_F^2 + \|\mathbf{B}_t - \mathbf{B}^*\|_F^2$$

$$- 2\alpha_t \left( \frac{\mu L}{\mu + L} \|\mathbf{B}_t - \mathbf{B}^*\|^2 + \frac{1}{\mu + L} \|\nabla f(\mathbf{B}_t) - \nabla f(\mathbf{B}^\star)\|_F^2 \right)$$

$$\leq \left( 1 - \frac{2\alpha_t \mu L}{\mu + L} \right) \|\mathbf{B}_t - \mathbf{B}^*\|_F^2 + \alpha_t^2 \|\nabla f(\mathbf{B}_t)\|_F^2 - \frac{2\alpha_t}{\mu + L} \|\nabla f(\mathbf{B}_t) - \nabla f(\mathbf{B}^\star)\|_F^2.$$

By Young's inequality,

$$\|\nabla f(\mathbf{B}_t)\|_F^2 \leq 2\|\nabla f(\mathbf{B}_t) - \nabla f(\mathbf{B}^\star)\|_F^2 + 2\|\nabla f(\mathbf{B}^\star)\|_F^2.$$

Thus the above yields, given that $\alpha_t \leq 1/(\mu + L)$,

$$\|\bar{\mathbf{B}}_t - \mathbf{B}^*\|_F^2 \leq \left( 1 - \frac{2\alpha_t \mu L}{\mu + L} \right) \|\mathbf{B}_t - \mathbf{B}^*\|_F^2 + 2\alpha_t \left( \alpha_t - \frac{1}{\mu + L} \right) \|\nabla f(\mathbf{B}_t) - \nabla f(\mathbf{B}^\star)\|_F^2$$

$$+ 2\alpha_t^2 \|\nabla f(\mathbf{B}^\star)\|_F^2$$

$$\leq \left( 1 - \frac{2\alpha_t \mu L}{\mu + L} \right) \|\mathbf{B}_t - \mathbf{B}^*\|_F^2 + 2\alpha_t^2 \|\nabla f(\mathbf{B}^\star)\|_F^2.$$

Since the projection onto the convex constraint set $\mathcal{B}$, $\Pi_{\mathcal{B}}$, is non-expansive,

$$\|\mathbf{B}_{t+1} - \mathbf{B}^*\|_F^2 \leq \|\bar{\mathbf{B}}_t - \mathbf{B}^*\|_F^2 \leq (1 - \alpha_t \rho) \|\mathbf{B}_t - \mathbf{B}^*\|_F^2 + \alpha_t^2 \sigma^2,$$

where we have denoted $\rho := \frac{2\mu L}{\mu + L}$ and $\sigma^2 := 2\|\nabla f(\mathbf{B}^\star)\|_F^2$.

Now, we show **(i)** (47). Suppose $\alpha_t \equiv \alpha > 0$. Note that $\mathbf{B}^*$ is a unique minimizer of $f$ over $\mathcal{B}$ because of strong convexity. Since $-\nabla f(\mathbf{B}^\star)$ lies in the normal cone of $\mathcal{B}$ at $\mathbf{B}^\star$, we have $\mathbf{B}^\star = \Pi_{\mathcal{B}}(\mathbf{B}^\star - \alpha \nabla f(\mathbf{B}^\star))$. Using non-expansiveness of convex projection $\Pi_{\mathcal{B}}$,

$$\|\mathbf{B}_{t+1} - \mathbf{B}^*\|_F = \|\Pi_{\mathcal{B}}(\mathbf{B}_t - \alpha \nabla f(\mathbf{B}_t)) - \Pi_{\mathcal{B}}(\mathbf{B}^* - \alpha \nabla f(\mathbf{B}^*))\|_F$$

$$\leq \|(\mathbf{B}_t - \alpha \nabla f(\mathbf{B}_t)) - (\mathbf{B}^* - \alpha \nabla f(\mathbf{B}^*))\|_F$$

$$= \left\| \int_0^1 (I - \alpha \nabla^2 f(\mathbf{B}_t + s(\mathbf{B}^\star - \mathbf{B}_t)))(\mathbf{B}_t - \mathbf{B}^\star) \, ds \right\|_F$$

$$\leq \sup_{\tilde{\mathbf{B}}} \|I - \alpha \nabla^2 f(\tilde{\mathbf{B}})\|_2 \|\mathbf{B}_t - \mathbf{B}^\star\|_F$$

$$\leq \underbrace{\max\{|1 - \alpha L|, |1 - \alpha \mu|\}}_{=:\rho(\alpha)} \|\mathbf{B}_t - \mathbf{B}^\star\|_F.$$

For the last inequality, since the eigenvalues of $\nabla^2 f(\tilde{\mathbf{B}})$ are contained in the interval $[\mu, L]$, the eigenvalues of $I - \alpha \nabla^2 f(z)$ are between $\min(1 - \alpha L, 1 - \alpha \mu)$ and $\max(1 - \alpha L, 1 - \alpha \mu)$. Note that $\rho(\alpha)$ is minimized at $\alpha = \frac{2}{\mu + L}$ with minimum value $\frac{L - \mu}{L + \mu}$. It follows that for any $t \geq 0$,

$$\|\mathbf{B}_{t+1} - \mathbf{B}^*\|_F \leq \rho(\alpha)^{t+1} \|\mathbf{B}_0 - \mathbf{B}^\star\|_F.$$

Next, we show **(ii)** (48) by induction. Indeed, it holds for $t = 1$ by the choice of $\nu$. Denoting $\hat{t} := \gamma + t$, by the induction hypothesis and by the choices of $\alpha_t = \beta / \hat{t}$ and $\nu \geq \frac{\beta^2 \sigma^2}{\beta \rho - 1}$,

$$\|\mathbf{B}_{t+1} - \mathbf{B}^\star\|_F^2 \leq \left( 1 - \frac{\beta \rho}{\hat{t}} \right) \frac{\nu}{\hat{t}} + \frac{\beta^2 \sigma^2}{\hat{t}^2}$$

$$\leq \left( \frac{\hat{t} - 1}{\hat{t}^2} \right) \nu \underbrace{- \left( \frac{\beta \rho - 1}{\hat{t}^2} \right) \nu + \frac{\beta^2 \sigma^2}{\hat{t}^2}}_{\leq 0}$$

$$\leq \frac{\hat{t} - 1}{(\hat{t} - 1)(\hat{t} + 1)} \nu = \frac{\nu}{\gamma + t + 1}.$$

This shows **(ii)**. $\qquad\square$

Next, we establish the following global landscape result of the ITR learning problem (6) by computing the Hessian of the objective $\mathcal{L}(\mathbf{B})$ in (6).

**Lemma E.3** (Local landscape of the ITR objective). *Let $\mathcal{L}(\mathbf{B})$ denote the ITR objective in (6).*

**(i)** *Suppose Assumption 3.1 holds. For the continuous outcome,*

$$(\lambda^- + \lambda_2)\mathbf{I}_{p(K-1)} \preceq \nabla_{\mathrm{vec}(\mathbf{B})}\nabla_{\mathrm{vec}(\mathbf{B})^T}\mathcal{L}(\mathbf{B}) \preceq (\lambda^+ + \lambda_2)\mathbf{I}_{p(K-1)}, \qquad (49)$$

*where the constants $\lambda^\pm$ are defined in Assumption 3.1.*

**(ii)** *Further Assumptions 3.1 and 3.2 hold. For the binary outcome,*

$$(\alpha^-\lambda^- + \lambda_2)\mathbf{I}_{p(K-1)} \preceq \nabla_{\mathrm{vec}(\mathbf{B})}\nabla_{\mathrm{vec}(\mathbf{B})^T}\mathcal{L}(\mathbf{B}) \preceq (\alpha^+\lambda^+ + \lambda_2)\mathbf{I}_{p(K-1)}. \qquad (50)$$

*where the constants $\lambda^\pm$ and $\alpha^\pm$ are defined in Assumptions 3.1 and 3.2, respectively.*

To maintain the flow, we write down the proof of Lemma E.3 to Section F.

Now we can easily deduce Theorem E.1.

***Proof of Theorem E.1.*** Lemma E.3 shows that the eigenvalues of the Hessian $\nabla_{\mathrm{vec}(\mathbf{B})}\nabla_{\mathrm{vec}(\mathbf{B})^T}\mathcal{L}$ of the ITR objective in (6) are uniformly bounded between constants $\mu$ and $L$ as defined in (10). Under Assumptions 3.1 and 3.2, we have $0 < \mu \leq L < \infty$. This shows that $\mathcal{L}$ is $\mu$-strongly convex and $L$-smooth over $\mathcal{B}$, as claimed in Theorem E.1 **(i)**.

Next, noting that $\|\cdot\|_F \leq \|\cdot\|_1$, triangle inequality and the fact that $\hat{\mathbf{B}}_0, \mathbf{B}^\star \in \mathcal{B}$ yields that, for any $t \geq 0$,

$$\|\mathbf{B}_t - \mathbf{B}^\star\|_F^2 \leq (\|\mathbf{B}_t\|_F + \|\mathbf{B}_\star\|_F)^2 \qquad (51)$$

$$\leq (\|\mathbf{B}_t\|_1 + \|\mathbf{B}_\star\|_1)^2 \qquad (52)$$

$$\leq 4\lambda_1^2. \qquad (53)$$

Hence the assertions **(i)**-**(ii)** follow directly from combining Lemmas E.2 and E.3. $\qquad\square$

# F  Proof of Lemma E.3

In this section, we prove Lemma E.3. We will first compute the gradient and the Hessian of the ITR objective in (6) separately for the continuous and binary outcome. We will then prove Lemma E.3 at the end of the section.

## F.1  Continuous outcome

**Proposition F.1.** *Let $\mathcal{L}(\mathbf{B})$ denote the ITR objective in (6) with continuous outcome. Then*

$$\nabla_{\mathbf{B}}\,\mathcal{L}(\mathbf{B}) = \frac{1}{n}\sum_{i=1}^{n}\omega(a_i, \mathbf{x}_i)\left(\mathbf{x}_i\mathbf{x}_i^T\mathbf{B}\mathbf{u}_{a_i}\mathbf{u}_{a_i}^T - \frac{K}{K-1}y_i\mathbf{x}_i\mathbf{u}_{a_i}^T\right) + \lambda_2\mathbf{B}, \qquad (54)$$

$$\nabla_{\mathrm{vec}(\mathbf{B})}\nabla_{\mathrm{vec}(\mathbf{B})^T}\mathcal{L}(\mathbf{B}) = \boldsymbol{\Psi} + \lambda_2\,\mathbf{I}_{p(K-1)}, \qquad (55)$$

*where $\boldsymbol{\Psi}$ is the weighted design matrix defined at (8).*

*Proof.* Recall that the ITR objective for continuous outcome is given by

$$\mathcal{L}(\mathbf{B}) = \frac{1}{2n}\sum_{i=1}^{n}\omega(a_i, \mathbf{x}_i)\left(\frac{K}{K-1}y_i - \mathbf{u}_{a_i}^T\mathbf{B}^T\mathbf{x}_i\right)^2 + \frac{\lambda_2}{2}\|\mathbf{B}\|_F^2.$$

It is trivial to show (54). For the Hessian, using (16),

$$
\nabla^2_{\mathbf{B}} \, \mathcal{L}(\mathbf{B}) = \nabla_{\text{vec}(\mathbf{B})} \nabla_{\text{vec}(\mathbf{B})^T} \, \mathcal{L}(\text{vec}(\mathbf{B}))
$$

$$
= \nabla_{\text{vec}(\mathbf{B})} \, \text{vec} \left( \frac{1}{n} \sum_{i=1}^{n} \omega(a_i, \mathbf{x}_i) \left( \mathbf{x}_i \mathbf{x}_i^T \mathbf{B} \mathbf{u}_{a_i} \mathbf{u}_{a_i}^T - \frac{K}{K-1} y_i \mathbf{x}_i \mathbf{u}_{a_i}^T \right) \right)^T + \lambda_2 \nabla^2_{\mathbf{B}}(\mathbf{B})
$$

$$
= \nabla_{\text{vec}(\mathbf{B})} \, \text{vec} \left( \frac{1}{n} \sum_{i=1}^{n} \omega(a_i, \mathbf{x}_i) \left( \mathbf{x}_i \mathbf{x}_i^T \mathbf{B} \mathbf{u}_{a_i} \mathbf{u}_{a_i}^T \right) \right)^T + \lambda_2 \, \mathbf{I}_{p(K-1)}
$$

$$
= \nabla_{\text{vec}(\mathbf{B})} \left[ \frac{1}{n} \sum_{i=1}^{n} \omega(a_i, \mathbf{x}_i) \left( \left( (\mathbf{u}_{a_i} \mathbf{u}_{a_i}^T)^T \otimes (\mathbf{x}_i \mathbf{x}_i^T) \right) \text{vec}(\mathbf{B}) \right)^T \right] + \lambda_2 \, \mathbf{I}_{p(K-1)}
$$

$$
= \nabla_{\text{vec}(\mathbf{B})} \left[ \frac{1}{n} \sum_{i=1}^{n} \omega(a_i, \mathbf{x}_i) \, \text{vec}(\mathbf{B})^T \left( (\mathbf{u}_{a_i} \mathbf{u}_{a_i}^T)^T \otimes (\mathbf{x}_i \mathbf{x}_i^T) \right)^T \right] + \lambda_2 \, \mathbf{I}_{p(K-1)}
$$

$$
= \frac{1}{n} \sum_{i=1}^{n} \omega(a_i, \mathbf{x}_i) \left( (\mathbf{u}_{a_i} \mathbf{u}_{a_i}^T)^T \otimes (\mathbf{x}_i \mathbf{x}_i^T) \right)^T + \lambda_2 \, \mathbf{I}_{p(K-1)}
$$

$$
= \frac{1}{n} \sum_{i=1}^{n} \omega(a_i, \mathbf{x}_i)(\mathbf{u}_{a_i} \mathbf{u}_{a_i}^T) \otimes (\mathbf{x}_i \mathbf{x}_i^T) + \lambda_2 \, \mathbf{I}_{p(K-1)}
$$

$$
= \boldsymbol{\Psi} + \lambda_2 \, \mathbf{I}_{p(K-1)}
$$

This shows the assertion. □

## F.2  Binary outcome

Given the current model parameter $\mathbf{B} \in \mathbb{R}^{p \times (K-1)}$, define the scalars $z_i, p_i, w_i$ for the $i$-th sample by

$$
z_i := \mathbf{u}_{\mathbf{a}_i}^T \mathbf{B}^T \mathbf{x}_i, \qquad p_i := \frac{\exp(z_i)}{1 + \exp(z_i)}, \qquad w_i := w(a_i, \mathbf{x}_i). \tag{56}
$$

For each binary label $y \in \{0, 1\}$ and activation $z \in \mathbb{R}$, recall the negative log-likelihood of observing $y$ under the logistic model $\text{Ber}(p_i)$ is given by

$$
\ell(y, z) = \log(1 + \exp(z)) - yz.
$$

An easy computation shows

$$
\dot{\mathbf{h}}(y, z) := \frac{\partial}{\partial z} \ell(y, z) = \frac{\exp(z)}{1 + \exp(z)} - y, \tag{57}
$$

$$
\ddot{\mathbf{h}}(y, z) := \frac{\partial^2}{\partial z^2} \ell(y, z) = \left( \frac{\exp(z)}{1 + \exp(z)} \right) \left( 1 - \frac{\exp(z)}{1 + \exp(z)} \right) \leq \frac{1}{4}. \tag{58}
$$

For the forthcoming computations, define matrices

$$
\mathbf{K} := [\dot{\mathbf{h}}(y_1, z_1), \dots, \dot{\mathbf{h}}(y_n, z_n)] \in \mathbb{R}^{1 \times n}, \tag{59}
$$

$$
\mathbf{M} := \text{diag} \left( \ddot{\mathbf{h}}(y_1, z_1), \dots, \ddot{\mathbf{h}}(y_n, z_n) \right) \in \mathbb{R}^{n \times n}, \tag{60}
$$

$$
\boldsymbol{\Phi} := [\mathbf{u}_{a_1} \otimes \mathbf{x}_1, \dots, \mathbf{u}_{a_n} \otimes \mathbf{x}_n] \in \mathbb{R}^{p(K-1) \times n}, \tag{61}
$$

$$
\mathbf{W} := \text{diag}(w_1, \dots, w_n) \in \mathbb{R}^{n \times n}. \tag{62}
$$

**Proposition F.2.** *Let $\mathcal{L}(\mathbf{B})$ denote the ITR objective in (6) with binary outcome. Suppose Assumption 3.1 holds. Let $z_i := \mathbf{u}_{a_i}^T \mathbf{B}^T \mathbf{x}_i$ for $i = 1, \dots, n$. Then*

$$
\nabla_{\mathbf{B}} \mathcal{L}(\mathbf{B}) = \frac{1}{n} \left( \sum_{i=1}^{n} w_i \dot{\mathbf{h}}(y_i, z_i) \mathbf{x}_i \mathbf{u}_{a_i}^T \right) + \lambda_2 \mathbf{B}, \tag{63}
$$

$$
\nabla_{\text{vec}(\mathbf{B})} \nabla_{\text{vec}(\mathbf{B})^T} \mathcal{L}(\mathbf{B}) = \frac{1}{n} \boldsymbol{\Phi}(\mathbf{M}\mathbf{W}) \boldsymbol{\Phi}^T + \lambda_2 \mathbf{I}_{p(K-1)} \tag{64}
$$

*Proof.* We first claim that

$$\nabla_{\mathrm{vec}(\mathbf{B})} z_i = \mathbf{C}^{(K-1,p)}(\mathbf{x}_i \otimes \mathbf{u}_{a_i}), \tag{65}$$

$$\nabla_{\mathrm{vec}(\mathbf{B})} \ell(y_i, z_i) = \mathbf{C}^{(K-1,p)}(\mathbf{x}_i \otimes \mathbf{u}_{a_i})\,\dot{\mathbf{h}}(y_i, z_i), \tag{66}$$

$$\nabla_{\mathrm{vec}(\mathbf{B})} \mathrm{vec}(\mathbf{K})^T = \nabla_{\mathrm{vec}(\mathbf{B})}\mathbf{K} = \mathbf{\Phi}\mathbf{M}. \tag{67}$$

Observe that by using (18), we can write

$$z_i = \mathrm{vec}(z_i) = \mathrm{vec}(\mathbf{u}_{a_i}^T \mathbf{B}^T \mathbf{x}_i) = (\mathbf{x}_i^T \otimes \mathbf{u}_{a_i}^T)\,\mathrm{vec}(\mathbf{B}^T) = \mathrm{vec}(\mathbf{B}^T)^T(\mathbf{x}_i \otimes \mathbf{u}_{a_i}).$$

Noting that $\mathrm{vec}(\mathbf{B}^T)^T = (\mathbf{C}^{(p,K-1)}\,\mathrm{vec}(\mathbf{B}))^T = \mathrm{vec}(\mathbf{B})^T\mathbf{C}^{(K-1,p)}$, Hence we get

$$\nabla_{\mathrm{vec}(\mathbf{B})} z_i = \nabla_{\mathrm{vec}(\mathbf{B})}\,\mathrm{vec}(\mathbf{B}^T)^T(\mathbf{x}_i \otimes \mathbf{u}_{a_i})$$
$$= \mathbf{C}^{(K-1,p)}(\mathbf{x}_i \otimes \mathbf{u}_{a_i}).$$

This verifies (65). Then by using the chain rule (20), we get

$$\nabla_{\mathrm{vec}(\mathbf{B})}\ell(y_i, z_i) = \nabla_{\mathrm{vec}(\mathbf{B})} z_i\,\nabla_{z_i}\ell(y_i, z_i)$$
$$= \mathbf{C}^{(K-1,p)}(\mathbf{x}_i \otimes \mathbf{u}_{a_i})\,\dot{\mathbf{h}}(y_i, z_i),$$

which verifies (66).

Next, we compute the gradients of $\mathbf{K}$ in (63). First, using (65) and the chain rule (20),

$$\nabla_{\mathrm{vec}(\mathbf{B})}\,\dot{\mathbf{h}}(y_i, z_i) = \nabla_{\mathrm{vec}(\mathbf{B})} z_i\,\nabla_{z_i}\dot{\mathbf{h}}(y_i, z_i)$$
$$= \mathbf{C}^{(K-1,p)}(\mathbf{x}_i \otimes \mathbf{u}_{a_i})\ddot{\mathbf{h}}(y_i, z_i)$$
$$= (\mathbf{u}_{a_i} \otimes \mathbf{x}_i)\ddot{\mathbf{h}}(y_i, z_i),$$

where the last equality uses commutation matrix relation (17). Then noting that $\mathrm{vec}(\mathbf{K})^T = \mathbf{K} = [\dot{\mathbf{h}}(y_1, z_1), \ldots, \dot{\mathbf{h}}(y_n, z_n)]$, it follows that

$$\nabla_{\mathrm{vec}(\mathbf{B})}\,\mathrm{vec}(\mathbf{K})^T \overset{(a)}{=} \left[(\mathbf{u}_{a_1} \otimes \mathbf{x}_1)\ddot{\mathbf{h}}(y_1, z_1), \ldots, (\mathbf{u}_{a_n} \otimes \mathbf{x}_n)\ddot{\mathbf{h}}(y_n, z_n)\right]$$

$$\overset{(b)}{=} [(\mathbf{u}_{a_1} \otimes \mathbf{x}_1), \ldots, (\mathbf{u}_{a_n} \otimes \mathbf{x}_n)]\,\mathrm{diag}\left(\ddot{\mathbf{h}}(y_1, z_1), \ldots, \ddot{\mathbf{h}}(y_n, z_n)\right)$$

$$\overset{(c)}{=} \mathbf{\Phi}\mathbf{M},$$

where (a) follows from (65) and the chain rule, (b) is an algebra, and (c) follows from the definition. This shows (67).

Now we compute the gradient:

$$\nabla_{\mathrm{vec}(\mathbf{B})}\left[\frac{1}{n}\sum_{i=1}^{n} w_i\,\ell(y_i, z_i)\right] \overset{(a)}{=} \frac{1}{n}\sum_{i=1}^{n} w_i\left[\mathbf{C}^{(K-1,p)}(\mathbf{x}_i \otimes \mathbf{u}_{a_i})\,\dot{\mathbf{h}}(y_i, z_i)\right] \tag{68}$$

$$\overset{(b)}{=} \frac{1}{n}\sum_{i=1}^{n} w_i(\mathbf{u}_{a_i} \otimes \mathbf{x}_i)\,\dot{\mathbf{h}}(y_i, z_i) \tag{69}$$

$$= \frac{1}{n}\left[w_1(\mathbf{u}_{a_1} \otimes \mathbf{x}_1), \ldots, w_n(\mathbf{u}_{a_n} \otimes \mathbf{x}_n)\right]\begin{bmatrix}\dot{\mathbf{h}}(y_1, z_1) \\ \vdots \\ \dot{\mathbf{h}}(y_n, z_n)\end{bmatrix} \tag{70}$$

$$\overset{(c)}{=} \frac{1}{n}\mathbf{\Phi}\mathbf{W}\,\mathrm{vec}(\mathbf{K}), \tag{71}$$

where (a) follows from (66), (b) uses the commutation matrix relation (17), and (c) follows from the definition. Then by using (69) and (18) with the fact that $\dot{\mathbf{h}}$ is a scalar, we deduce (63).

Next, we compute the Hessian. From (71),

$$\nabla_{\mathrm{vec}(\mathbf{B})}\mathcal{L}(\mathbf{B}) = \frac{1}{n}\mathbf{\Phi}\mathbf{W}\,\mathrm{vec}(\mathbf{K}) + \lambda_2\,\mathrm{vec}(\mathbf{B}). \tag{72}$$

Then using (66) with (72), we get

$$\nabla_{\mathrm{vec}(\mathbf{B})}\nabla_{\mathrm{vec}(\mathbf{B})^T}\mathcal{L}(\mathbf{B}) = \nabla_{\mathrm{vec}(\mathbf{B})}\left[\frac{1}{n}\,\mathrm{vec}(\mathbf{K})^T\mathbf{W}\mathbf{\Phi}^T + \lambda_2\mathbf{I}_{p(K-1)}\right] \tag{73}$$

$$= \frac{1}{n}\mathbf{\Phi}(\mathbf{M}\mathbf{W})\mathbf{\Phi}^T + \lambda_2\mathbf{I}_{p(K-1)}. \tag{74}$$

This shows the assertion. $\qquad\square$

### F.3  Proof of Lemma E.3

We are now ready to give proof of Lemma E.3.

***Proof of Lemma E.3.*** Recall that $\mathbf{\Phi} := [\mathbf{u}_{a_1}\otimes\mathbf{x}_1,\dots,\mathbf{u}_{a_n}\otimes\mathbf{x}_n]$. Note that

$$\frac{1}{n}\mathbf{\Phi}\mathbf{W}\mathbf{\Phi}^T = \frac{1}{n}\sum_{i=1}^{n} w_i(\mathbf{u}_{a_i}\otimes\mathbf{x}_i)(\mathbf{u}_{a_i}\otimes\mathbf{x}_i)^T \tag{75}$$

$$= \frac{1}{n}\sum_{i=1}^{n} w_i(\mathbf{u}_{a_i}\mathbf{u}_{a_i}^T\otimes\mathbf{x}_i\mathbf{x}_i^T) \tag{76}$$

$$= \mathbf{\Psi}, \tag{77}$$

where $\mathbf{\Psi}$ is the weighted design matrix defined at (8). (In particular, since $\mathbf{W}$ is a diagonal matrix of nonnegative entries, this shows that $\mathbf{\Psi}$ is positive semi-definite.)

According to Propositions F.1 and F.2, we have

$$\nabla_{\mathrm{vec}(\mathbf{B})}\nabla_{\mathrm{vec}(\mathbf{B})^T}\mathcal{L}(\mathbf{B}) = \begin{cases} n^{-1}\mathbf{\Phi}\mathbf{W}\mathbf{\Phi}^T + \lambda_2\mathbf{I}_{p(K-1)} & \text{for continuous outcome} \\ n^{-1}\mathbf{\Phi}\mathbf{M}\mathbf{W}\mathbf{\Phi}^T + \lambda_2\mathbf{I}_{p(K-1)} & \text{for binary outcome.} \end{cases} \tag{78}$$

Thus the assertion for the continuous outcome follows immediately. For the case of a binary outcome, by Assumption 3.2 we have

$$\alpha^-\mathbf{I}_n \preceq \mathbf{M} \preceq \alpha^+\mathbf{I}_n. \tag{79}$$

Then

$$\alpha^-\left(\frac{1}{n}\mathbf{\Phi}\mathbf{W}\mathbf{\Phi}^T\right) \preceq \frac{1}{n}\mathbf{\Phi}(\mathbf{M}\mathbf{W})\mathbf{\Phi}^T \preceq \alpha^+\left(\frac{1}{n}\mathbf{\Phi}\mathbf{W}\mathbf{\Phi}^T\right). \tag{80}$$

It follows that

$$\alpha^-\lambda_{\min}\left(\frac{1}{n}\mathbf{\Phi}\mathbf{W}\mathbf{\Phi}^T\right)\mathbf{I}_{p(K-1)} \preceq \frac{1}{n}\mathbf{\Phi}(\mathbf{M}\mathbf{W})\mathbf{\Phi}^T \preceq \alpha^+\lambda_{\max}\left(\frac{1}{n}\mathbf{\Phi}\mathbf{W}\mathbf{\Phi}^T\right)\mathbf{I}_{p(K-1)} \tag{81}$$

This shows the assertion for the binary outcome holds. $\qquad\square$

# G   A non-asymptotic consistency of weighted, constrained, and regularized MLE

In this section, we provide a general result on the non-asymptotic consistency of MLE in a general setting. Here, observations are assumed to be i.i.d. from a generative model, and the unknown true parameter of the generative model may lie on the boundary of the parameter space.

A generic sample $(X, Y)$ consists of a random covariate $X$ and outcome $Y$. (In the ITR setting, we will specialize $X$ to be the pair of the patient's covariate and the corresponding treatment.) We assume the covariate $X$ follows a probability distribution $\mu(\cdot)$ and the conditional distribution of $Y$ given $X = x$ is parameterized as $\pi_{\boldsymbol{\theta}_\star}(\cdot \,|\, x)$ for an unknown parameter $\boldsymbol{\theta}_\star$. Hence the generic random sample $(X, Y)$ follows the parameterized distribution $\pi_{\boldsymbol{\theta}_\star}(y \,|\, x)\mu(x)$. Assume we know a smooth parametric family of conditional distributions $\boldsymbol{\theta} \mapsto \pi_{\boldsymbol{\theta}}(\cdot \,|\, x)$, where the true parameter $\boldsymbol{\theta}_\star$ lies in a convex parameter space $\boldsymbol{\Theta} \subseteq \mathbb{R}^d$. Our goal is to estimate $\boldsymbol{\theta}_\star$ from i.i.d. samples $(x_i, y_i)$, $i = 1, \ldots, n$ drawn from $\pi_{\boldsymbol{\theta}_\star}(\cdot \,|\, \cdot)\mu(\cdot)$.

To estimate $\boldsymbol{\theta}_\star$, a natural approach is to maximize the empirical joint likelihood function. Since the marginal distribution of the covariate $x$ is independent of the parameter $\boldsymbol{\theta}$, we can apply the method of "reweighting" the covariate distribution by introducing an additional random weight $\omega(x)$ that we can control. Namely, fix a weighting function $\omega(\cdot) : \mathcal{X} \to [0, \infty)$, where $\mathcal{X}$ is covariate space, that assigns a nonnegative weight to each covariate value $x$. Consider then the following weighted expected log-likelihood maximization problem:

$$\arg\min_{\boldsymbol{\theta} \in \boldsymbol{\Theta}} \mathbb{E}_{(X,Y)} \left[ -\omega(X) \log \pi_{\boldsymbol{\theta}_\star}(Y \,|\, X) \right], \tag{82}$$

where we omit the marginal distribution $\mu(X)$ as it is independent of $\boldsymbol{\theta}$.

Recall that we get to design the weighting function $\omega(\cdot)$. We first observe that the true parameter $\boldsymbol{\theta}_\star$ is a stationary point of the weighted expected log-likelihood function $\mathbb{E}\left[-\omega(X) \log \pi_{\boldsymbol{\theta}_\star}(Y \,|\, X)\right]$ for arbitrary weighting function $\omega$. This can be justified under the mild assumption of exchangeability of the expectation and derivatives (which holds under twice continuous differentiability of the log-likelihood function by the dominated convergence theorem). Indeed, note that

$$\mathbb{E}_{(X,Y)} \left[\omega(X) \nabla_{\boldsymbol{\theta}} \log \pi_{\boldsymbol{\theta}}(Y \,|\, X)\right] = \mathbb{E}_{(X,Y)} \left[\omega(X) \frac{\nabla_{\boldsymbol{\theta}} \pi_{\boldsymbol{\theta}}(Y \,|\, X)}{\pi_{\boldsymbol{\theta}}(Y \,|\, X)}\right]$$

$$= \mathbb{E}_X \left[\mathbb{E}_Y \left[\omega(X) \frac{\nabla_{\boldsymbol{\theta}} \pi(Y \,|\, X)}{\pi_{\boldsymbol{\theta}}(Y \,|\, X)} \,\Big|\, X\right]\right]$$

$$= \mathbb{E}_X \left[\omega(X) \mathbb{E}_{Y \sim \pi_{\boldsymbol{\theta}_\star}(\cdot \,|\, X)} \left[\frac{\nabla_{\boldsymbol{\theta}} \pi_{\boldsymbol{\theta}}(Y|X)}{\pi_{\boldsymbol{\theta}}(Y \,|\, X)} \,\Big|\, X\right]\right] = 0,$$

where the last equality follows from

$$\mathbb{E}_{Y \sim \pi_{\boldsymbol{\theta}_\star}(\cdot \,|\, X)} \left[\frac{\nabla_{\boldsymbol{\theta}} \pi_{\boldsymbol{\theta}_\star}(Y|X)}{\pi_{\boldsymbol{\theta}_\star}(Y \,|\, X)} \,\Big|\, X\right] = \int \nabla_{\boldsymbol{\theta}} \pi_{\boldsymbol{\theta}_\star}(y \,|\, X) \, dy$$

$$= \nabla_{\boldsymbol{\theta}} \int \pi_{\boldsymbol{\theta}_\star}(y \,|\, X) \, dy$$

$$= \nabla_{\boldsymbol{\theta}} 1 = 0.$$

Therefore, $\boldsymbol{\theta}_\star$ is a critical point of $\mathbb{E}\left[\omega(X) \log \pi_{\boldsymbol{\theta}}(Y \,|\, X)\right]$. In particular, if we assume that the log-likelihood function $\log \pi_{\boldsymbol{\theta}}(y \,|\, x)\mu(x)$ is convex in $\boldsymbol{\theta}$, then $\boldsymbol{\theta}_\star$ is a global maximizer of $\mathbb{E}\left[\omega(X) \log \pi_{\boldsymbol{\theta}}(Y \,|\, X)\right]$.

Next, we introduce the finite-sample approximation of (82). Denote $w_i := \omega(x_i)$ for the weight of the $i$th sample $(x_i, y_i)$. Then the weighted negative log-likelihood of the observed samples under the model parameter $\boldsymbol{\theta}$ is

$$\mathcal{L}(\boldsymbol{\theta}) := -\frac{1}{n} \sum_{i=1}^n w_i \underbrace{\log \pi_{\boldsymbol{\theta}}(y_i \,|\, \mathbf{x}_i)}_{=: \mathcal{L}_0(\boldsymbol{\theta})} + R(\boldsymbol{\theta}), \tag{83}$$

where $R(\boldsymbol{\theta})$ is a suitable choice of regularizer for parameter $\boldsymbol{\theta}$. The regularization term can be considered as prior knowledge about the model parameter $\boldsymbol{\theta}$.

Let $\hat{\boldsymbol{\theta}}_n$ denote a (possibly non-unique) minimizer of the above function over $\boldsymbol{\Theta}$. This is a minimizer of the loss function $\mathcal{L}$ over the constraint set $\boldsymbol{\Theta}$, which we call the *weighted, constrained, and regularized MLE* of $\boldsymbol{\theta}_\star$.

As in the standard MLE analysis, we assume that the negative log-likelihood function $\boldsymbol{\theta} \mapsto \mathcal{L}_0(\boldsymbol{\theta})$ is differentiable and strictly convex for all $x \in \mathbb{R}^p$. Statistical literature typically assumes that the true parameter lies in the interior of the constrained parameter space, $\boldsymbol{\Theta}$, a convex subset of $\mathbb{R}^d$. In contrast, we allow the true parameter $\boldsymbol{\theta}_\star$ to lie on the boundary of $\boldsymbol{\Theta}$. Regardless of that, we have seen above that the gradient of the weighted expected log-likelihood function vanishes at the true parameter $\boldsymbol{\theta}_\star$.

In this setting, we aim to provide a high-probability guarantee that there exists a global minimizer of (83) that is close to the true parameter $\boldsymbol{\theta}_\star$ *for an arbitrary weighting function* $\omega(\cdot)$, which we will optimize later for the best statistical guarantee. In Theorem G.1 below, we obtain such a result in the non-asymptotic, weighted, constrained, and regularized setting. For its proof, we combine a classical approach from [15] with concentration inequalities, namely, a classical Berry-Esseen bound for deviations from the standard normal distribution for independent but non-identically distributed random variables and a uniform McDirmid bound (Lemma G.2). The Berry-Esseen bound controls the linear term in the second-order Taylor expansion of the log-likelihood function ($T_n(\boldsymbol{\theta})$ in (96)), while the McDirmid bound controls the quadratic term ($S_n(\boldsymbol{\theta})$ in (96)). By using an $\varepsilon$-net argument, the latter concentration inequality can be extended to a setting where the random variables are parameterized within a compact set.

**Theorem G.1** (Non-asymptotic consistency of weighted, constrained, and regularized MLE). *Consider the constrained, regularized, and weighted MLE problem* (83) *with unknown parameter* $\boldsymbol{\theta}_\star$ *from a convex subset* $\boldsymbol{\Theta} \subseteq \mathbb{R}^d$. *Fix an arbitrary weighting function* $\omega(\cdot)$. *Assume the following holds:*

**(a1)** *(Smoothness) For each sample* $(x, y)$, *the per-sample negative log-likelihood function* $\boldsymbol{\theta} \mapsto \log \pi_{\boldsymbol{\theta}}(y \mid x)\mu(x)$ *is strictly convex, three-times continuously differentiable, and* $R(\boldsymbol{\theta})$ *is differentiable. Furthermore, denote* $U := \omega(X)\nabla_{\boldsymbol{\theta}} \log \pi_{\boldsymbol{\theta}_\star}(Y \mid X) \in \mathbb{R}^d$ *and* $\overline{U} := U - \mathbb{E}[U]$. *Suppose there are constants* $D_1, d_1 \in (0, \infty)$ *such that for all* $i$,

$$\mathbb{E}[\|\overline{U}\|^3] < D_1, \qquad \min_{1 \le k \le d} \mathrm{Var}(U(k)) > d_1. \tag{84}$$

**(a2)** *(First-order optimality) The true parameter* $\boldsymbol{\theta}_\star$ *is a stationary point of the expected negative weighted log-likelihood function* $\overline{\mathcal{L}}_0(\boldsymbol{\theta}) := \mathbb{E}_{\boldsymbol{\theta}_\star}[-\omega(X) \log \pi_{\boldsymbol{\theta}}(Y \mid X)]$ *over* $\boldsymbol{\Theta}$:

$$\langle \nabla_{\boldsymbol{\theta}} \overline{\mathcal{L}}_0(\boldsymbol{\theta}_\star), \boldsymbol{\theta} - \boldsymbol{\theta}_\star \rangle \ge 0 \quad \forall \boldsymbol{\theta} \in \boldsymbol{\Theta}. \tag{85}$$

**(a3)** *(Approximate second-order optimality) Let* $\bar{\mathcal{L}}(\boldsymbol{\theta}) := \overline{\mathcal{L}}_0(\boldsymbol{\theta}) + R(\boldsymbol{\theta})$ *denote the expected regularized negative log- likelihood function. Then the regularized Fisher information* $\nabla^2 \bar{\mathcal{L}}(\boldsymbol{\theta})$ *is positive definite at* $\boldsymbol{\theta} = \boldsymbol{\theta}_\star$ *with minimum eigenvalue* $\rho > 0$.

*Fix a constant* $C > 0$. *Let* $D = Cn^{-1/2} + \frac{8\|\nabla R(\boldsymbol{\theta}_\star)\|}{\rho}$ *and* $M = M(D) > 0$ *denote the supremum of the absolute values of all third-order partial derivatives of* $\mathcal{L}$ *over all* $\boldsymbol{\theta}$ *with* $\|\boldsymbol{\theta} - \boldsymbol{\theta}_\star\| \le D$. *Suppose* $\|\nabla R(\boldsymbol{\theta}_\star)\|$ *is small enough so that*

$$D \le \frac{3\rho}{4M(D)}. \tag{86}$$

*Then there are constants* $c_1, c_2, c_3 > 0$ *such that*

$$\mathbb{P}\left( \left\| \boldsymbol{\theta}_\star - \underset{\boldsymbol{\theta} \in \boldsymbol{\Theta}}{\arg\min}\, \mathcal{L}(\boldsymbol{\theta}) \right\| \le D \right) \ge 1 - c_1 \exp\left( -\frac{C^2 \rho^2}{2 \cdot 32^2 d} \right) - \frac{c_2 \sum_{i=1}^n w_i^3}{\left( \sum_{i=1}^n w_i^2 \right)^{3/2}}$$
$$- O(\exp(-c_3 n)). \tag{87}$$

*That is, with high probability explicitly depending on* $C$, $\rho$, $d$, *and* $n$, *there exists a global minimizer of* $\boldsymbol{\theta} \mapsto \mathcal{L}(\boldsymbol{\theta})$ *in* $\boldsymbol{\Theta}$ *within distance* $D$ *from* $\boldsymbol{\theta}_\star$.

It is important to notice that the minimum eigenvalue $\rho$ of the weighted regularized Fisher information $\nabla^2 \overline{\mathcal{L}}(\boldsymbol{\theta})$ depends on the choice of weighting function $\omega(\cdot)$. Therefore, Theorem G.1 suggests that

it is best to choose the weighting function to maximize the minimum eigenvalue of the weighted regularized Fisher information in order to minimize the statistical estimation error. We elaborate more on this point in the proof of Theorem 3.5.

We devote the rest of this section to proving Theorem G.1 using Lemma G.2 and Theorem G.3.

**Lemma G.2** (A uniform McDirmid's inequality). *Let $\mathbf{X}_1, \ldots, \mathbf{X}_n$ be independent random vectors in $\mathbb{R}^p$ from a joint distribution $\pi$. Fix a compact parameter space $\boldsymbol{\Theta} \subseteq \mathbb{R}^d$ and $f_{\boldsymbol{\theta}} : \mathbb{R}^d \to [-J, J]$ is a bounded functional for each $\boldsymbol{\theta} \in \boldsymbol{\Theta}$ such that*

$$\|f_{\boldsymbol{\theta}} - f_{\boldsymbol{\theta}'}\|_\infty \leq L\|\boldsymbol{\theta} - \boldsymbol{\theta}'\|, \qquad \forall \boldsymbol{\theta}, \boldsymbol{\theta}' \in \boldsymbol{\Theta} \tag{88}$$

*for some constant $L > 0$. Further assume that $\mathbb{E}[f_{\boldsymbol{\theta}}(\mathbf{X}_k)] = 0$ for all $\boldsymbol{\theta} \in \boldsymbol{\Theta}$ and $k = 1, \ldots, n$. Then there exist constant $J > 0$ such that for each $n \geq 0$, and $\eta > 0$,*

$$\mathbb{P}\left(\sup_{\boldsymbol{\theta} \in \boldsymbol{\Theta}} \left| \frac{1}{n} \sum_{k=1}^n f_{\boldsymbol{\theta}}(\mathbf{X}_k) \right| \geq \eta \right) \leq \left( \frac{2L \operatorname{diam}(\boldsymbol{\Theta})}{\eta} \right)^d \exp\left( -\frac{\eta^2 n}{2J^2} \right). \tag{89}$$

*Proof.* Since $\boldsymbol{\Theta} \subseteq \mathbb{R}^d$ is compact, it can be covered by a finite number of $L_2$-balls of any given radius $\varepsilon > 0$. Let $\mathcal{U}_\varepsilon$ be such an open cover using the least number of balls of radius $\varepsilon > 0$. Let $N(\varepsilon) = |\mathcal{U}_\varepsilon|$ denote the least number of such balls to cover $\boldsymbol{\Theta}$. Moreover, let $\operatorname{diam}(\boldsymbol{\Theta})$ denote the diameter of $\boldsymbol{\Theta}$, which is finite since $\boldsymbol{\Theta}$ is compact. Then $\boldsymbol{\Theta}$ is contained in a $d$-dimensional box of side length $\operatorname{diam}(\boldsymbol{\Theta})$. This box can be covered by $(\operatorname{diam}(\boldsymbol{\Theta})/\varepsilon)^d$ cubes of side length $\varepsilon$. Covering each such cube of side length $\varepsilon$ by a ball of radius $\varepsilon$, it follows that

$$N(\varepsilon) \leq \left( \frac{\operatorname{diam}(\boldsymbol{\Theta})}{\varepsilon} \right)^d. \tag{90}$$

Next, fix $\eta > 0$, $\boldsymbol{\theta} \in \boldsymbol{\Theta}$, and $\varepsilon > 0$. Let $\boldsymbol{\theta}_1, \cdots, \boldsymbol{\theta}_{N(\varepsilon)}$ be the centers of balls in the open cover $\mathcal{U}_\varepsilon$. Then there exists $1 \leq j \leq N(\varepsilon)$ such that $\|\boldsymbol{\theta} - \boldsymbol{\theta}_j\| < \varepsilon$. By the hypothesis, $f_{\boldsymbol{\theta}}$ depends on $\boldsymbol{\theta}$ uniformly continuously with respect to the supremum norm. Hence there exists $\delta = \delta(\varepsilon) > 0$ such that

$$\|f_{\boldsymbol{\theta}} - f_{\boldsymbol{\theta}_j}\|_\infty \leq L\varepsilon. \tag{91}$$

Denote $H_n(\boldsymbol{\theta}) := n^{-1} \sum_{k=1}^n f_{\boldsymbol{\theta}}(\mathbf{X}_k)$. Then this yields, almost surely,

$$|H_n(\boldsymbol{\theta}) - H_n(\boldsymbol{\theta}_j)| \leq L\varepsilon. \tag{92}$$

Furthermore, by the assumption, $\|f_{\boldsymbol{\theta}}\|_\infty$ is uniformly bounded by $J > 0$. It follows that for each $\boldsymbol{\theta} \in \boldsymbol{\Theta}$, $H_n(\boldsymbol{\theta})$ changes its value at most by $M$ when one of $\mathbf{X}_1, \ldots, \mathbf{X}_n$ is replaced arbitrarily. Therefore, by the standard McDirmid's inequality (see Theorem 2.9.1. in [51]) and a union bound, with choosing $\varepsilon = \eta/(2L)$,

$$\mathbb{P}\left(|H_n(\boldsymbol{\theta})| \geq \eta\right) \leq \sum_{j=1}^{N(\eta/2L)} \mathbb{P}\left(|H_n(\boldsymbol{\theta}_j)| \geq \eta/2\right) \leq K \left( \frac{2L \operatorname{diam}(\boldsymbol{\Theta})}{\eta} \right)^d \exp\left( -\frac{\eta^2 n}{2J^2} \right). \tag{93}$$

The above holds for all $n \geq 1$ and $\eta > 0$. This shows the assertion. $\square$

Next, we recall the classical Berry-Esseen theorem for the rate of convergence of normal approximation for the sum of independent random variables due to Feller.

**Theorem G.3** (Berry-Esseen, Theorem 2 in Ch. XVI.5 of Feller '91[17]). *Let $X_1, X_2, \ldots, X_n$ be independent and not necessarily identically distributed random variables with zero means and finite variances. Define $W = \sum_{i=1}^n X_i$ let $F$ be the distribution function of $W$ and $\Phi$ be the standard normal distribution function. Then*

$$\|F - \Phi\|_\infty \leq \frac{6 \sum_{i=1}^n \mathbb{E}[|X_i|^3]}{\left( \sum_{i=1}^n \mathbb{E}[X_i^2] \right)^{3/2}}. \tag{94}$$

Now we prove Theorem G.1.

***Proof of Theorem G.1.*** Suppose we have $n$ i.i.d. observed samples $(x_i, y_i)$ for $i = 1, \ldots, n$ from distribution $\pi_{\boldsymbol{\theta}_\star}(y \,|\, x)\mu(x)$. Also by the assumption, $\mathcal{L}_0$ is twice continuously differentiable and $n^{-1} \sum_{i=1}^n w_i = 1$, so $\mathbb{E}[\nabla \mathcal{L}_0] = \nabla \mathbb{E}[\mathcal{L}_0]$ and $\mathbb{E}[\nabla^2 \mathcal{L}_0] = \nabla^2 \mathbb{E}[\mathcal{L}_0]$ by the dominated convergence theorem.

Note that, for any $r > 0$,

$$\left\{ \left\| \boldsymbol{\theta}_\star - \arg\min_{\boldsymbol{\theta} \in \boldsymbol{\Theta}} \mathcal{L}(\boldsymbol{\theta}) \right\| \leq r \right\} = \left\{ \inf_{\substack{\boldsymbol{\theta} \in \boldsymbol{\Theta} \\ \|\boldsymbol{\theta} - \boldsymbol{\theta}_\star\| = r}} \mathcal{L}(\boldsymbol{\theta}) - \mathcal{L}(\boldsymbol{\theta}_\star) > 0 \right\}. \tag{95}$$

The inclusion $\subseteq$ is clear. Conversely, since $\mathcal{L}$ is strictly convex, the event on the right implies that its unique global minimizer is within $r$ from $\boldsymbol{\theta}_\star$, as desired. Thus in order to show the main result in (87), it suffices to show that the probability of the event on the right with $r = D = Cn^{-1/2} + \frac{8\|\nabla R(\boldsymbol{\theta}_\star)\|}{\rho}$ is at least the right-hand side of (87).

Fix $\boldsymbol{\theta} \in \boldsymbol{\Theta}$ such that $\|\boldsymbol{\theta} - \boldsymbol{\theta}_\star\| = D$. We introduce two random variables that we will bound to be small by using some concentration inequalities:

$$T_n(\boldsymbol{\theta}) := \frac{\sqrt{n}}{\|\boldsymbol{\theta} - \boldsymbol{\theta}_\star\|} \left\langle \nabla_{\boldsymbol{\theta}} \mathcal{L}_0(\boldsymbol{\theta}_\star) - \mathbb{E}\left[\omega(X) \nabla_{\boldsymbol{\theta}} \mathcal{L}_0(\boldsymbol{\theta}_\star)\right], \boldsymbol{\theta} - \boldsymbol{\theta}_\star \right\rangle, \tag{96}$$

$$S_n(\boldsymbol{\theta}) := \frac{1}{\|\boldsymbol{\theta} - \boldsymbol{\theta}_\star\|^2} (\boldsymbol{\theta} - \boldsymbol{\theta}_\star)^T \left(\nabla_{\boldsymbol{\theta}} \nabla_{\boldsymbol{\theta}^T} \mathcal{L}(\boldsymbol{\theta}_\star) - \nabla_{\boldsymbol{\theta}} \nabla_{\boldsymbol{\theta}^T} \left(\mathbb{E}\left[\omega(X)\mathcal{L}(\boldsymbol{\theta}_\star)\right]\right)\right)(\boldsymbol{\theta} - \boldsymbol{\theta}_\star). \tag{97}$$

Roughly speaking, these quantities are of order 1 with high probability, $T_n$ by Central Limit Theorem and $S_n$ by the law of large numbers. Later in this proof, we will use non-asymptotic versions of these classical limit theorems to obtain high probability bounds for related quantities.

Since $\boldsymbol{\theta} \mapsto \mathcal{L}(\boldsymbol{\theta})$ is assumed to be three-time continuously differentiable, the quantity $M \geq 0$ in the assertion is well-defined and is finite. Then, using the Taylor expansion, we may write

$$\mathcal{L}(\boldsymbol{\theta}) - \mathcal{L}(\boldsymbol{\theta}_\star) \geq \langle \nabla_{\boldsymbol{\theta}} \mathcal{L}(\boldsymbol{\theta}_\star), \boldsymbol{\theta} - \boldsymbol{\theta}_\star \rangle + \frac{1}{2}(\boldsymbol{\theta} - \boldsymbol{\theta}_\star)^T \nabla_{\boldsymbol{\theta}} \nabla_{\boldsymbol{\theta}^T} \mathcal{L}(\boldsymbol{\theta}_\star)(\boldsymbol{\theta} - \boldsymbol{\theta}_\star)$$
$$- \frac{M(\|\boldsymbol{\theta} - \boldsymbol{\theta}_\star\|)}{6} \|\boldsymbol{\theta} - \boldsymbol{\theta}_\star\|^3. \tag{98}$$

We will lower bound the first two terms on the right-hand side above. Note that

$$\langle \nabla_{\boldsymbol{\theta}} \mathcal{L}(\boldsymbol{\theta}_\star), \boldsymbol{\theta} - \boldsymbol{\theta}_\star \rangle = \langle \nabla_{\boldsymbol{\theta}} \mathcal{L}_0(\boldsymbol{\theta}_\star), \boldsymbol{\theta} - \boldsymbol{\theta}_\star \rangle - \mathbb{E}\left[\omega(X) \langle \nabla_{\boldsymbol{\theta}} \mathcal{L}_0(\boldsymbol{\theta}_\star), \boldsymbol{\theta} - \boldsymbol{\theta}_\star \rangle\right]$$
$$+ \langle \nabla_{\boldsymbol{\theta}} \mathbb{E}[\omega(X)\mathcal{L}_0(\boldsymbol{\theta}_\star)], \boldsymbol{\theta} - \boldsymbol{\theta}_\star \rangle + \langle \nabla R(\boldsymbol{\theta}_\star), \boldsymbol{\theta} - \boldsymbol{\theta}_\star \rangle$$
$$\overset{(a)}{\geq} \langle \nabla_{\boldsymbol{\theta}} \mathcal{L}_0(\boldsymbol{\theta}_\star), \boldsymbol{\theta} - \boldsymbol{\theta}_\star \rangle - \mathbb{E}\left[\omega(X) \langle \nabla_{\boldsymbol{\theta}} \mathcal{L}_0(\boldsymbol{\theta}_\star), \boldsymbol{\theta} - \boldsymbol{\theta}_\star \rangle\right]$$
$$- \|\nabla R(\boldsymbol{\theta}_*)\| \cdot \|\boldsymbol{\theta} - \boldsymbol{\theta}_*\|$$
$$\overset{(b)}{=} - \frac{\|\boldsymbol{\theta} - \boldsymbol{\theta}_\star\|}{\sqrt{n}} T_n(\boldsymbol{\theta}) - \|\nabla R(\boldsymbol{\theta}_*)\| \cdot \|\boldsymbol{\theta} - \boldsymbol{\theta}_*\|,$$

where for (a) we use the fact that $\boldsymbol{\theta}_\star$ is a stationary point of $\mathbb{E}[\mathcal{L}_0(\boldsymbol{\theta})]$ over $\boldsymbol{\Theta}$ and Cauchy-Schwarz inequality; for (b) we use the definition of $T_n(\boldsymbol{\theta})$.

Next, we turn our attention to the second-order term in the Taylor expansion (98). Recall that from the assumption **(a3)** in Theorem G.1,

$$\mathbb{E}\left[\omega(X) \nabla_{\boldsymbol{\theta}} \nabla_{\boldsymbol{\theta}^T} \mathcal{L}(\boldsymbol{\theta}_\star)\right] = \nabla_{\boldsymbol{\theta}} \nabla_{\boldsymbol{\theta}^T} \left(\mathbb{E}\left[\omega(X)\mathcal{L}(\boldsymbol{\theta}_\star)\right]\right) \succeq \rho \mathbf{I}_p, \tag{99}$$

where $\rho > 0$ is a constant. It follows that

$$(\boldsymbol{\theta} - \boldsymbol{\theta}_\star)^T \nabla_{\boldsymbol{\theta}} \nabla_{\boldsymbol{\theta}^T} \mathcal{L}(\boldsymbol{\theta}_\star)(\boldsymbol{\theta} - \boldsymbol{\theta}_\star) \tag{100}$$
$$\geq (\boldsymbol{\theta} - \boldsymbol{\theta}_\star)^T \left[\nabla_{\boldsymbol{\theta}} \nabla_{\boldsymbol{\theta}^T} \mathcal{L}(\boldsymbol{\theta}_\star) - \nabla_{\boldsymbol{\theta}} \nabla_{\boldsymbol{\theta}^T} \left(\mathbb{E}\left[\omega(X)\mathcal{L}(\boldsymbol{\theta}_\star)\right]\right)\right](\boldsymbol{\theta} - \boldsymbol{\theta}_\star) + \rho\|\boldsymbol{\theta} - \boldsymbol{\theta}_\star\|^2 \tag{101}$$
$$\geq \|\boldsymbol{\theta} - \boldsymbol{\theta}_\star\|^2 (S_n(\boldsymbol{\theta}) + \rho). \tag{102}$$

Combining the above inequalities with noting that $\|\boldsymbol{\theta} - \boldsymbol{\theta}_\star\|^2 = D^2$, we obtain

$$
\frac{\mathcal{L}(\boldsymbol{\theta}) - \mathcal{L}(\boldsymbol{\theta}_\star)}{\|\boldsymbol{\theta} - \boldsymbol{\theta}_\star\|^2}
$$

$$
\geq \frac{1}{\|\boldsymbol{\theta} - \boldsymbol{\theta}_\star\|^2} \left[ \langle \nabla_{\boldsymbol{\theta}} \mathcal{L}(\boldsymbol{\theta}_\star), \boldsymbol{\theta} - \boldsymbol{\theta}_\star \rangle + \frac{1}{2}(\boldsymbol{\theta} - \boldsymbol{\theta}_\star)^T \nabla_{\boldsymbol{\theta}} \nabla_{\boldsymbol{\theta}^T} \mathcal{L}(\boldsymbol{\theta}_\star)(\boldsymbol{\theta} - \boldsymbol{\theta}_\star) \right.
$$

$$
\left. - \frac{M(\|\boldsymbol{\theta} - \boldsymbol{\theta}_\star\|)}{6} \|\boldsymbol{\theta} - \boldsymbol{\theta}_\star\|^3 \right]
$$

$$
\geq -\frac{1}{\|\boldsymbol{\theta} - \boldsymbol{\theta}_\star\|} \left[ \frac{T_n(\boldsymbol{\theta})}{\sqrt{n}} + \|\nabla R(\boldsymbol{\theta}_\star)\| \right] + \frac{1}{2}(S_n(\boldsymbol{\theta}) + \rho) - \frac{M(\|\boldsymbol{\theta} - \boldsymbol{\theta}_\star\|)}{6} \|\boldsymbol{\theta} - \boldsymbol{\theta}_\star\|
$$

$$
= \frac{1}{\|\boldsymbol{\theta} - \boldsymbol{\theta}_\star\|} \underbrace{\left( -\|\nabla R(\boldsymbol{\theta}_\star)\| + \frac{\rho}{4} D - \frac{M(D)}{6} D^2 \right)}_{=:I_1} + \underbrace{\left( \frac{1}{2}\left( S_n(\boldsymbol{\theta}) + \frac{\rho}{2} \right) - \frac{1}{\sqrt{n}D} T_n(\boldsymbol{\theta}) \right)}_{=:I_2}.
$$

Note that $I_1 \geq 0$ if

$$
\frac{\rho}{8} D \geq \|\nabla R(\boldsymbol{\theta}_\star)\| \quad \text{and} \quad \frac{\rho}{8} D \geq \frac{M(D)}{6} D^2.
$$

The choice of $D$ holds the former condition, and the latter by the assumption (86). Thus, $I_1 \geq 0$.

We now take infimum over all $\boldsymbol{\theta} \in \Theta$ such that $\|\boldsymbol{\theta} - \boldsymbol{\theta}_\star\| = D$. It suffices to show that the random variable $I_2$ defined above is positive with high probability, since $I_1$ does not depend on $\boldsymbol{\theta}$. To this end, write

$$
\inf_{\substack{\boldsymbol{\theta} \in \Theta \\ \|\boldsymbol{\theta} - \boldsymbol{\theta}_\star\| = D}} I_2 \geq \underbrace{\left( \inf_{\substack{\boldsymbol{\theta} \in \Theta \\ \|\boldsymbol{\theta} - \boldsymbol{\theta}_\star\| = D}} -\frac{1}{\sqrt{n}D} T_n(\boldsymbol{\theta}) \right)}_{=:A} + \underbrace{\left( \inf_{\substack{\boldsymbol{\theta} \in \Theta \\ \|\boldsymbol{\theta} - \boldsymbol{\theta}_\star\| = D}} \frac{1}{2}\left( S_n(\boldsymbol{\theta}) + \frac{\rho}{2} \right) \right)}_{=:B}. \tag{103}
$$

Then the last expression in (103) is at least $\rho/16$ if $A \geq -\rho/16$ and $B \geq \rho/8$. By the hypothesis, $D = O(1)$ so it is uniformly bounded. Then by the uniform McDirmid's inequality in Lemma G.2, there exist constants $C', C'' > 0$ such that

$$
\mathbb{P}\left( B < \frac{\rho}{8} \right) = \mathbb{P}\left( \inf_{\substack{\boldsymbol{\theta} \in \Theta \\ \|\boldsymbol{\theta} - \boldsymbol{\theta}_\star\| = D}} S_n(\boldsymbol{\theta}) < -\frac{\rho}{4} \right) \leq D^d C' \exp(-C'' n). \tag{104}
$$

Next, we will show the following inequalities: For $K = 6D_1/d_1^{3/2}$,

$$
\mathbb{P}\left( A < -\frac{\rho}{16} \right) = \mathbb{P}\left( \inf_{\substack{\boldsymbol{\theta} \in \Theta \\ \|\boldsymbol{\theta} - \boldsymbol{\theta}_\star\| = D}} T_n(\boldsymbol{\theta}) \geq \frac{\sqrt{n}D\rho}{16} \right)
$$

$$
\overset{(c)}{\leq} d \left( \mathbb{P}\left( Z \geq \frac{\sqrt{n}D\rho}{32\sqrt{d}} \right) + \frac{K \sum_{i=1}^n w_i^3}{\sqrt{n}\left( \sum_{i=1}^n w_i^2 \right)^{3/2}} \right)
$$

$$
\overset{(d)}{\leq} d \left( \frac{32\sqrt{d}}{C\rho\sqrt{2\pi}} \exp\left( -\frac{nD^2\rho^2}{2 \cdot 32^2 d} \right) + \frac{K \sum_{i=1}^n w_i^3}{\sqrt{n}\left( \sum_{i=1}^n w_i^2 \right)^{3/2}} \right)
$$

$$
\overset{(e)}{\leq} d \left( \frac{32\sqrt{d}}{C\rho\sqrt{2\pi}} \exp\left( -\frac{C^2\rho^2}{2 \cdot 32^2 d} \right) + \frac{K \sum_{i=1}^n w_i^3}{\sqrt{n}\left( \sum_{i=1}^n w_i^2 \right)^{3/2}} \right), \tag{105}
$$

where $Z \sim N(0,1)$ is an independent standard normal random variable. (d) above is a simple consequence of the standard Gaussian tail bound $\mathbb{P}(N(0,1) > x) \leq \frac{e^{-x^2/2}}{x\sqrt{2\pi}}$ and (d), (e) are from the choice of $D$ which yields $nD^2 \geq C^2$. We will show (c) at the end of this proof. Note that for any two events $E_1, E_2$ defined on the same probability space, $\mathbb{P}(E_1 \cap E_2) \geq \mathbb{P}(E_1) + \mathbb{P}(E_2) - 1$. Then

by combining (103), (104), and (105), we have

$$
\begin{aligned}
\mathbb{P}&\left( \inf_{\substack{\boldsymbol{\theta}\in\Theta \\ \|\boldsymbol{\theta}-\boldsymbol{\theta}_\star\|=D}} \frac{\mathcal{L}(\boldsymbol{\theta})-\mathcal{L}(\boldsymbol{\theta}_\star)}{\|\boldsymbol{\theta}-\boldsymbol{\theta}_\star\|^2} \geq \frac{\rho}{16}\right)\\
&\geq \mathbb{P}\left(A\geq -\frac{\rho}{16}\right) + \mathbb{P}\left(B\geq\frac{\rho}{8}\right) - 1\\
&\geq 1 - d\left(\frac{32\sqrt{d}}{C\rho\sqrt{2\pi}}\exp\left(-\frac{C^2\rho^2}{2\cdot 32^2 d}\right)+\frac{K\sum_{i=1}^n w_i^3}{\left(\sum_{i=1}^n w_i^2\right)^{3/2}}\right)+\left(1-D^d C'\exp(-C''n)\right)-1\\
&= 1 - c_1\exp\left(-\frac{C^2\rho^2}{2\cdot 32^2 d}\right)-\frac{c_2\sum_{i=1}^n w_i^3}{\left(\sum_{i=1}^n w_i^2\right)^{3/2}}-O(\exp(-c_3 n)),
\end{aligned}
$$

with the corresponding constants $c_1, c_2, c_3 > 0$. Then the assertion (87) will follow using (95).

It remains to verify (c) in (105). To this end, write

$$
\nabla_{\boldsymbol{\theta}}\mathcal{L}_0(\boldsymbol{\theta}_\star) - \mathbb{E}\left[\omega(X)\nabla_{\boldsymbol{\theta}}\mathcal{L}_0(\boldsymbol{\theta}_\star)\right] = -\frac{1}{n}\sum_{i=1}^n w_i\overline{U}_i,
$$

where $U_i = \omega(X_i)\nabla_{\boldsymbol{\theta}}\log\pi_{\boldsymbol{\theta}_\star}(Y_i\,|\,X_i)\in\mathbb{R}^d$ and $\overline{U}_i = U_i - \mathbb{E}[U_i]$ as in the statement. Then by using Cauchy-Schwarz inequality and noting that $\|\boldsymbol{\theta}-\boldsymbol{\theta}_\star\| = D$, we get

$$
T_n(\boldsymbol{\theta}) = \left\langle -\frac{1}{\sqrt{n}}\sum_{i=1}^n w_i\overline{U}_i,\ \frac{\boldsymbol{\theta}-\boldsymbol{\theta}_\star}{D}\right\rangle \leq \left\|\frac{1}{\sqrt{n}}\sum_{i=1}^n w_i\overline{U}_i\right\| = \sqrt{\sum_{k=1}^d \left|\frac{1}{\sqrt{n}}\sum_{i=1}^n w_i\overline{U}_i(k)\right|^2}.
$$

It is important to note that the distribution of the random variable on the last term above does not depend on $\boldsymbol{\theta}$. Note that $w_i\overline{U}_i$ for $i=1,\dots,n$ are independent mean zero (but possibly non-identical distributions due to the weights $w_i$) random vectors in $\mathbb{R}^d$, and by the hypothesis, their respective coordinates have uniformly bounded variances. Hence by a union bound,

$$
\mathbb{P}\left( \inf_{\substack{\boldsymbol{\theta}\in\Theta \\ \|\boldsymbol{\theta}-\boldsymbol{\theta}_\star\|=D}} T_n(\boldsymbol{\theta})\geq t\right) \leq \sum_{k=1}^d \mathbb{P}\left(\left|\frac{1}{\sqrt{n}}\sum_{i=1}^n w_i\overline{U}_i(k)\right|\geq \frac{t}{\sqrt{d}}\right). \tag{106}
$$

Let $Q_n^k = \frac{1}{\sqrt{n}}\sum_{i=1}^n w_i\overline{U}_i(k)$. Then by the Berry-Esseen Theorem (Theorem G.3) and the hypothesis, for $Z\sim N(0,1)$,

$$
\begin{aligned}
\sup_{z\in\mathbb{R}}\left|\mathbb{P}\left(Q_n^k\leq z\right)-\mathbb{P}\left(Z\leq z\right)\right| &\leq \frac{6\sum_{i=1}^n w_i^3 \mathbb{E}[\|\overline{U}_i\|^3]}{\left(\sum_{i=1}^n w_i^2 \mathrm{Var}(\overline{U}_i(k))\right)^{3/2}}\\
&= \frac{6\,\mathbb{E}[\|\overline{U}_1\|^3]\sum_{i=1}^n w_i^3}{\left(\mathrm{Var}(\overline{U}_1(k))\right)^{3/2}\left(\sum_{i=1}^n w_i^2\right)^{3/2}}\\
&\leq \frac{6D_1\sum_{i=1}^n w_i^3}{d_1^{3/2}\left(\sum_{i=1}^n w_i^2\right)^{3/2}}\\
&= \frac{K\sum_{i=1}^n w_i^3}{\left(\sum_{i=1}^n w_i^2\right)^{3/2}} \quad \text{for } k=1,\dots,d.
\end{aligned}
$$

with $K = 6D_1/d_1^{3/2}$. Combining with (106) and using a triangle inequality, we obtain

$$
\mathbb{P}\left( \inf_{\substack{\boldsymbol{\theta}\in\Theta \\ \|\boldsymbol{\theta}-\boldsymbol{\theta}_\star\|=D}} T_n(\boldsymbol{\theta})\geq t\right) \leq d\left(\mathbb{P}\left(Z\geq \frac{t}{2\sqrt{d}}\right)+\frac{K\sum_{i=1}^n w_i^3}{\left(\sum_{i=1}^n w_i^2\right)^{3/2}}\right). \tag{107}
$$

Thus (c) in (105) follows. $\qquad\square$

# H   Proof of Theorem 3.5

In this section, we provide proof of the computational and statistical estimation guarantee stated in Theorem (3.5) under the generative ITR model (11). Two key ingredients in our proof are the computational guarantee (Theorem. E.1) and non-asymptotic estimation guarantee of constrained and regularized MLE (Theorem. G.1).

As we have briefly explained in the main text, the connection between the weighted convex optimization (6) for ITR learning and the statistical estimation problem under the generative model (11) is established by writing down the corresponding maximum likelihood estimation problem. To see this, recall that covariate-treatment pair $(\mathbf{x}_i, a_i)$ follows a joint distribution $\pi$ that does not depend on the true parameter $\mathbf{B}_\star$. Further, assume that $\pi$ admits a density function $f_\pi$. Then, for the continuous setting, the joint log-likelihood of observing the triple $(\mathbf{x}_i, a_i, y_i)$ gives under the generative model with parameter $\mathbf{B}$

$$-\frac{1}{2\sigma^2} \sum_{i=1}^{n} \left( y_i - \frac{K-1}{K} \mathbf{u}_{a_i}^T \mathbf{B}^T \mathbf{x}_i \right)^2 + \sum_{i=1}^{n} \log f_\pi(\mathbf{x}_i, a_i) + \text{constant}.$$

Thus, the standard (unweighted) maximum likelihood estimation of the true parameter $\mathbf{B}_\star$ can be written as

$$\min_{\mathbf{B} \in \mathbb{R}^{p \times (K-1)}, \|\mathbf{B}\| \leq \lambda_1} \sum_{i=1}^{n} \left( \frac{K}{K-1} y_i - \mathbf{u}_{a_i}^T \mathbf{B}^T \mathbf{x}_i \right)^2,$$

where we imposed additional $L_1$-ball constraint for the parameter $\mathbf{B}$. Now, if we use covariate-balancing weight $w(\mathbf{x}_i, a_i)$ for each subject $i$, the corresponding weighted MLE problem reads as

$$\min_{\mathbf{B} \in \mathbb{R}^{p \times (K-1)}, \|\mathbf{B}\| \leq \lambda_1} \sum_{i=1}^{n} w(\mathbf{x}_i, a_i) \left( \frac{K}{K-1} y_i - \mathbf{u}_{a_i}^T \mathbf{B}^T \mathbf{x}_i \right)^2, \tag{108}$$

which is exactly the weighted convex optimization problem (6) for continuous outcome. A similar discussion applies to the binary outcome simply by noting that the joint log-likelihood is given by

$$\sum_{i=1}^{n} \left( -\log \left( 1 + \exp(\mathbf{u}_{a_i}^T \mathbf{B}^T \mathbf{x}_i) \right) + y_i \mathbf{u}_{a_i}^T \mathbf{B}^T \mathbf{x}_i \right) + \sum_{i=1}^{n} \log f_\pi(\mathbf{x}_i, a_i).$$

***Proof of Theorem 3.5.***   Consider the generative ITR model (11) with true parameter $\mathbf{B}_\star$. Fix $\varepsilon > 0$. Let $(\widehat{\mathbf{B}}_t)_{t \geq 0}$ denote the estimated parameters by using the PGD algorithm (7) for the weighted empirical maximum likelihood estimation problem (6). Decompose the total estimation error into computational and statistical parts as

$$\|\mathbf{B}_\star - \widehat{\mathbf{B}}_T\|_F \leq \underbrace{\|\widehat{\mathbf{B}} - \widehat{\mathbf{B}}_T\|_F}_{=:I_1} + \underbrace{\|\widehat{\mathbf{B}} - \widehat{\mathbf{B}}_\star\|_F}_{=:I_2}.$$

By Theorem E.1, we have

$$I_1 \leq \begin{cases} 4\rho(\alpha)^T \lambda_1^2 & \text{with constant stepsize} \\ \frac{\nu}{\gamma + T} & \text{with diminishing stepsize}. \end{cases}$$

We wish to apply our general result on weighted and constrained MLE in (Theorem G.1) with $L_2$-regularization $R(\mathbf{B}) = \frac{\lambda_2}{2} \|\mathbf{B}\|_F^2$ in order to get a high-probability bound on the term $I_2$. For this, we first verify the hypothesis in Theorem G.1 for our generative ITR model (11). Assumption **(a1)** in Theorem G.1 follows from Assumption 3.4. For Assumption **(a3)** in Theorem G.1, denoting by $\rho$ the minimum eigenvalue of the expected regularized Fisher information, by Lemma E.3 (also see (10)),

$$\rho = \mu = \begin{cases} \lambda^- + \lambda_2 & \text{for continuous outcome} \\ \alpha^- \lambda^- + \lambda_2 & \text{for binary outcome}. \end{cases}$$

where $\lambda_-$ is the minimum eigenvalue of the expected weighted design matrix $\mathbb{E}[\boldsymbol{\Psi}]$ in (8) (see Assumption 3.1). The above quantity is positive under the hypothesis.

Lastly, for Assumption **(a2)** in Theorem G.1, we need to show that the true parameter $\mathbf{B}_\star$ is a global maximizer of the expected log-likelihood function of our generative ITR model (11). This can be shown by a standard argument under the mild assumption that expectations and derivatives are exchangeable. For simplicity, we prove this for the case when $\mathbf{B}_\star$ is in the interior of the constraint set $\boldsymbol{\Theta}$. (The general case can be shown similarly by considering the first-order optimality condition for stationary points.) First note that, if we denote the likelihood of observing the triple $(X, A, Y)$ under the model parameter $\mathbf{B}$ as $p_\mathbf{B}(X, A, Y) = p_\mathbf{B}(Y \mid X, A)\, p(X, A)$, then $\log p_\mathbf{B}(X, A, Y)$ is a concave function in $\mathbf{B}$. Thus we only need to show that the true parameter $\mathbf{B}_\star$ is a critical point of the expected weighted log-likelihood function $\mathbb{E}\left[\omega(A, X) \log p_\mathbf{B}(X, A, Y)\right]$. Indeed,

$$
\begin{aligned}
\nabla_\mathbf{B} & \mathbb{E}\left[\omega(A, X) \log p_\mathbf{B}(X, A, Y)\right] \\
&= \mathbb{E}\left[\omega(A, X) \nabla_\mathbf{B} \left(\log p_\mathbf{B}(Y \mid X, A) + \log p(X, A)\right)\right] \\
&= \mathbb{E}\left[\omega(A, X) \nabla_\mathbf{B} \log p_\mathbf{B}(Y \mid X, A)\right] \\
&= \mathbb{E}\left[\omega(A, X) \frac{\nabla_\mathbf{B} p_\mathbf{B}(Y|X, A)}{p_\mathbf{B}(Y \mid X, A)}\right] \\
&= \mathbb{E}_X\left[\mathbb{E}_{(Y, A)}\left[\omega(A, X) \frac{\nabla_\mathbf{B} p_\mathbf{B}(Y|X, A)}{p_\mathbf{B}(Y \mid X, A)} \,\middle|\, X\right]\right] \\
&= \mathbb{E}_X\left[\sum_{a=0}^{K} \omega(a, X) \pi(a, X)\, \mathbb{E}_{Y \sim p_{\mathbf{B}_\star}(\cdot \mid X, a)}\left[\frac{\nabla_\mathbf{B} p_\mathbf{B}(Y|X, A)}{p_\mathbf{B}(Y \mid X, A)} \,\middle|\, X, A = a\right]\right],
\end{aligned}
$$

where we have used iterated expectation to condition first on the covariate $X$ and then integrate out the treatment-response pair $(A, Y)$. Now note that

$$
\mathbb{E}_{Y \sim p_{\mathbf{B}_\star}(\cdot \mid X, a)}\left[\frac{\nabla_\mathbf{B} p_{\mathbf{B}_\star}(Y|X, A)}{p_{\mathbf{B}_\star}(Y \mid X, A)} \,\middle|\, X, A = a\right] = \int \nabla_\mathbf{B} p_{\mathbf{B}_\star}(Y|X, A)\, dy \tag{109}
$$

$$
= \nabla_\mathbf{B} \int p_{\mathbf{B}_\star}(Y|X, A)\, dy \tag{110}
$$

$$
= \nabla_\mathbf{B} 1 = 0. \tag{111}
$$

Thus $\mathbf{B}_\star$ is a critical point of $\mathbb{E}\left[\log p_\mathbf{B}(X, A, Y)\right]$. By the concavity of the log-likelihood function, it follows that $\mathbf{B}_\star$ is a global maximizer of $\mathbb{E}\left[\omega(A, X) \log p_\mathbf{B}(X, A, Y)\right]$. In particular, this verifies the assumption G.1 **(a2)**.

Now we can apply Theorem G.1 with $L_2$-regularization $R(\mathbf{B}) = \frac{\lambda_2}{2}\|\mathbf{B}\|_F^2$ for our generative ITR model. Denoting

$$
D := \frac{C}{\sqrt{n}} + \frac{8\|\nabla R(\mathbf{B}_\star)\|_F}{\mu} = \frac{C}{\sqrt{n}} + \frac{8\lambda_2\|\mathbf{B}_\star\|_F}{\mu} \tag{112}
$$

for $C > 0$ any constant, this gives

$$
\mathbb{P}\left(I_2 > D\right) \le c_1 \exp\left(-\frac{C^2 \mu^2}{2 \cdot 32^2 d}\right) + \frac{c_2 \sum_{i=1}^{n} w_i^3}{\left(\sum_{i=1}^{n} w_i^2\right)^{3/2}} + O(\exp(-c_3 n)),
$$

where $c_1, c_2, c_3 > 0$ are constants and $d = p(K - 1)$. By Assumption 3.4, the quantities $\sqrt{n} \sum_{i=1}^{n} w_i^3 / \left(\sum_{i=1}^{n} w_i^2\right)^{3/2}$ are uniformly bounded in $n$. Hence we can first choose $n$ large enough so that the last two terms combined in the right-hand side in (113) are at most $\varepsilon/2$, for which $n \gtrsim \varepsilon^{-2}$ is sufficient. Then we can choose $C > 0$ large enough so that the first term on the right-hand side of (113) is at most $\varepsilon/2$. For this, we choose

$$
C = \frac{32\sqrt{2d}}{\mu} \sqrt{\log \frac{2c_1}{\varepsilon}} = \Omega\left(\frac{\sqrt{p \log \varepsilon^{-1}}}{\mu}\right)
$$

Accordingly, we can choose $T$ large enough so that $I_1 \le C/\sqrt{n}$. Then the above gives

$$
\mathbb{P}\left(\|\mathbf{B}_\star - \widehat{\mathbf{B}}_T\|_F > \frac{2C}{\sqrt{n}} + \frac{8\lambda_2\|\mathbf{B}_\star\|_F}{\mu}\right) \le \mathbb{P}\left(I_2 \ge D\right) \le \varepsilon. \tag{113}
$$

This shows the assertion. $\qquad\square$

# I   Additional simulation details and results

The code supporting this study is available at https://github.com/ljw9510/effective-ITR, with plans for release as an R package soon.

## I.1   Details of data-generating mechanism

In this section, we explain how data are generated and treatment assignments are allocated in our simulation studies. The outcome is generated under the working model assumption given by Equation 1 with heteroscedastic random noise $\epsilon \sim N(0, \sigma^2(A, \mathbf{X}))$. We have four different settings/cases for the treatment-free effect and interaction effect functions, which are outlined below.

**Case 1: Linear interaction functions with simple treatment-free effect functions**

For Case 1, our randomized study aligns with Scenario 7 and our observational study mirrors Scenario 8 described in [44]. Specifically, we consider the following settings for the treatment-free effects and interaction effects:

Randomized trial:
$$\mu(\mathbf{X}) = 1 + 2X_1 + 2X_2,$$
$$\delta(\mathbf{X}) = \begin{cases} 0.75 + 1.5X_1 + 1.5X_2 + 1.5X_3 + 1.5X_4, A = 1; \\ 0.75 + 1.5X_1 - 1.5X_2 - 1.5X_3 + 1.5X_4, A = 2; \\ 0.75 + 1.5X_1 - 1.5X_2 + 1.5X_3 - 1.5X_4, A = 3; \\ 0.75 - 1.5X_1 + 1.5X_2 - 1.5X_3 - 1.5X_4, A = 4, \end{cases}$$

Observational study:
$$\mu(\mathbf{X}) = 1 + X_5 + 3X_6 + 2X_1X_2,$$
$$\delta(\mathbf{X}) = \begin{cases} 0.5 + 2X_1 + X_2 + X_3, A = 1; & 1 + X_1 - X_2 - X_3, A = 2; \\ 1.5 + 3X_1 - X_2 + X_3, A = 3; & 1 - X_1 - X_2 + X_3, A = 4, \end{cases}$$

**Case 2: Linear interaction functions with complicated treatment-free effect functions**

For Case 2, we consider the following treatment-free effect function for model assumptions of $Y$:
$$\mu(\mathbf{X}) = 1 + X_5 + X_5^2 + 2e^{-X_1X_2} + \sin(X_3).$$

**Case 3: Non-linear interaction functions with simple treatment-free effect functions**

For Case 3, we consider the following non-linear interaction functions for model assumptions of $Y$:
$$\mu(\mathbf{X}) = 1 + 2X_1 + 2X_2 + 2X_4 - 2X_4^2 + 2X_1X_2,$$
$$\delta(\mathbf{X}) = \begin{cases} 0.5 + 1.0X_1 - 2.0X_4 + 0.5X_4^2, A = 1; \; 1.0 + 1.0X_1 + 1.0X_4 - 1.0X_4^2, A = 2; \\ 1.5 + 2.0X_1 - 1.0X_4 - 1.0X_4^2, A = 3; \; 1.0 - 1.0X_1 - 1.0X_4 - 1.0X_4^2, A = 4. \end{cases}$$

**Case 4: Non-linear interaction functions with a complicated treatment-free effect function**

For Case 4, we consider the following complicated treatment-free effect function for model assumptions of $Y$:
$$\mu(\mathbf{X}) = 1 + 2X_1 + 2X_2 + 2X_4 - 2X_4^2 + 2X_1X_2 + 2e^{-X_1X_2} + \sin(X_3).$$

In all four cases, we adopt the same treatment assignment principles and variance function $\sigma^2(A, \mathbf{X})$ that induces heteroscedastic error for observational studies as outlined in [44]:

$$\pi(A, \mathbf{X}) = \begin{cases} 0.25 \cdot \mathbf{I}(X_1 < 0) + 0.4 \cdot \mathbf{I}(X_1 > 0), A = 1, \\ 0.25 \cdot \mathbf{I}(X_1 < 0) + 0.2 \cdot \mathbf{I}(X_1 > 0), A = 2, 3, 4. \end{cases}$$
$$\sigma^2(A, \mathbf{X}) = 0.25 + 2X_2 \cdot \mathbf{I}(X_2 > 0) + X_3 \cdot \mathbf{I}(X_3 > 0, A = 1) + X_4 \cdot \mathbf{I}(X_4 > 0, A = 2).$$

For randomized trials, each individual has an equal probability of assigning each treatment, therefore, no propensity score estimation was made for AD-learning and SABD-learning. For the heteroscedastic error for randomized trials, $\sigma^2(A, \mathbf{X}) = 0.25 + 0.2\,(1.5 - X_2)^2$ was used.

## I.2  Implementation details

All numerical experiments were performed on a 2022 Macbook Air with M1 chip and 16 GB of RAM.

In the computation of the final ITR estimates, we use two different approaches: one with weighted multivariate regression with $L_1$-regularization for the penalized approach (named 'penalized' in the main text). The other uses the proposed PGD algorithm with $L_1$-ball (named 'constrained' in the main text). To ensure a fair comparison, both the penalized method and the constrained optimization algorithm were implemented using code developed by the author. We do not consider additional $L_2$-regularization in this setting. To determine the regularization constant or $L_1$-ball size, we use mean squared error as the criterion. The iterate number $T$ of the PGD algorithm is 1000.

In our simulation study, we employ the default configuration of the random forest algorithm as implemented in the `randomForest` package in R to estimate the residual variance function in SABD-Learning, the treatment-free effect, and the propensity score. This default configuration includes the construction of an ensemble of 500 decision trees, with each tree's splitting criterion determined using the Gini index. The number of variables considered for splitting at each node is set to the square root of the total number of covariates. For model performance evaluation, we utilize the *out-of-bag* (OOB) error estimate.

## I.3  Additional simulation results

The goal of these additional simulations in the appendix is to demonstrate, under the pre-specified linear ITR, that DCBW, variable screening, and augmentation can synergistically benefit the final ITR. Methods using energy balancing weights are denoted as "_e" and those employing the distance covariance test for variable screening are indicated as "_s" in the table. Note that the augmented "SABD_e_s" corresponds to " proposed+penalized" in the main text. In this section, we include the performance of various methods implemented only by the penalized approach to compare the effectiveness of the proposed statistical methods. The simulation was implemented using package *glmnet* in R.

### I.3.1  Treatment Decision Accuracy

Here, we show the averages and standard deviations of treatment decision accuracy from the additional simulation studies. The first four tables present the results of experiments that mimic randomized trials, and the remaining four tables show the results of observational studies.

The simulation results indicate that individual components alone do not significantly improve performance. Instead, the synergistic effect of combining these components is essential for the ITR-Learning. For example, in high-dimensional settings, using DCBW alone balances the empirical distributions of all covariates (up to 60), including a significant portion of irrelevant ones. Additionally, estimated decision functions may include irrelevant variables, reducing the impact of DCBW. Similarly, using variable screening alone remains challenging due to model misspecification or highly nonlinear outcomes. Combining other methods, such as outcome augmentation, helps mitigate model misspecification effects. This combined application ensures efficient and robust optimal ITR estimation across diverse scenarios.

Below is a detailed investigation of each scenario.

First, we examine Case 1, where the treatment rule class is correctly specified (Tables 1, 5). With a relatively simple treatment-free effect, AD-Learning's performance is comparable to SABD-Learning, particularly following outcome augmentation. When combined with EBWs as AD_e, SABD_e, these methods exhibit improved performance over their original versions (AD, SABD). In randomized trials, EBWs effectively reduce finite sample imbalance compared to IPW with true propensity scores. In observational studies, energy balancing yields more effective balancing weights, resulting in higher accuracy.

Next, we delve into Case 2, involving a more complicated treatment-free effect function compared to Case 1 (Tables 2, 6). Notably, since the treatment-free effect does not affect ITR learning, a misspecified treatment-free effect function introduces heterogeneity and increases noise variance. Then we can see the synergistic effectiveness of outcome augmentation and inverse variance weighting in SABD-Learning, effectively handling additional heterogeneity in ITR estimation (See "augmented" panel in Tables 2, 6). However, it's important to note that all variants of AD-Learning show subpar performance, even with augmented outcomes. AD-Learning results in an intercept-only model frequently as its estimated decision function, which suggests that the mean squared error criterion in LASSO regularization may not be effective for selecting models. In such instances, different penalty types or performance metrics could prove beneficial.

Case 3 and Case 4 handle situations where the underlying treatment rule class is misspecified (Tables 3, 4, 7, and 8). Particularly, in Case 4, estimating ITR becomes even more challenging due to a highly non-linear covariate-outcome relationship, which leads to suboptimal results of the variants of AD-Learning as in Case 2. Our results demonstrate that the more robust EBWs outperform standard IPW, particularly in observational studies. Additionally, variable screening enhances ITR learning in a high-dimensional setting. Employing variable screening (SABD_s) enhances the performance of SABD, especially with limited sample sizes, as it aids in selecting effect modifiers and precision variables, thereby facilitating decision function estimation. However, even with perfect screening, estimating the decision function remains challenging due to model misspecification or highly nonlinear outcomes. Simultaneous use of EBWs and screening for SABD (SABD_e_s) further elevates performance, likely influenced by dimensionality in nonparametric balancing weight methods.

The significant enhancements from integrating DCBWs, variable screening, outcome augmentation, and inverse variance weighting underscore their collective importance. Their combined application ensures efficient and robust optimal ITR estimation across diverse scenarios.

Table 2: Average accuracy and their standard deviations (in parenthesis) in the randomized trial in Case 1: Linear interaction functions with simple treatment-free effect functions.

| n | p | $AD$ | $AD_e$ | $AD_s$ | $AD_{e,s}$ | $SABD$ | $SABD_e$ | $SABD_s$ | $SABD_{e,s}$ |
|---|---|------|--------|--------|------------|--------|----------|----------|--------------|
| | | | | | | Original | | | |
| | 20 | 0.679 | 0.734 | 0.703 | 0.813 | 0.720 | 0.745 | 0.734 | 0.804 |
| | | (0.06) | (0.06) | (0.06) | (0.06) | (0.06) | (0.06) | (0.06) | (0.06) |
| 200 | 40 | 0.637 | 0.673 | 0.658 | 0.755 | 0.691 | 0.700 | 0.702 | 0.761 |
| | | (0.07) | (0.08) | (0.07) | (0.08) | (0.06) | (0.06) | (0.07) | (0.07) |
| | 60 | 0.633 | 0.649 | 0.644 | 0.720 | 0.671 | 0.686 | 0.685 | 0.739 |
| | | (0.07) | (0.08) | (0.06) | (0.08) | (0.06) | (0.07) | (0.05) | (0.06) |
| | 20 | 0.816 | 0.872 | 0.828 | 0.925 | 0.832 | 0.867 | 0.818 | 0.875 |
| | | (0.05) | (0.05) | (0.04) | (0.04) | (0.04) | (0.04) | (0.04) | (0.03) |
| 600 | 40 | 0.814 | 0.863 | 0.808 | 0.895 | 0.840 | 0.867 | 0.821 | 0.871 |
| | | (0.04) | (0.03) | (0.04) | (0.07) | (0.04) | (0.03) | (0.04) | (0.05) |
| | 60 | 0.802 | 0.848 | 0.783 | 0.876 | 0.836 | 0.859 | 0.815 | 0.864 |
| | | (0.04) | (0.03) | (0.04) | (0.06) | (0.04) | (0.03) | (0.04) | (0.03) |
| | 20 | 0.865 | 0.902 | 0.867 | 0.953 | 0.866 | 0.882 | 0.846 | 0.875 |
| | | (0.03) | (0.05) | (0.03) | (0.04) | (0.03) | (0.04) | (0.04) | (0.03) |
| 1000 | 40 | 0.854 | 0.900 | 0.846 | 0.927 | 0.866 | 0.893 | 0.839 | 0.876 |
| | | (0.03) | (0.03) | (0.03) | (0.07) | (0.04) | (0.03) | (0.04) | (0.04) |
| | 60 | 0.857 | 0.895 | 0.838 | 0.922 | 0.874 | 0.893 | 0.846 | 0.879 |
| | | (0.03) | (0.02) | (0.03) | (0.05) | (0.03) | (0.02) | (0.03) | (0.03) |
| | | | | | | Augmented | | | |
| | 20 | 0.798 | 0.807 | 0.806 | 0.853 | 0.807 | 0.807 | 0.820 | 0.854 |
| | | (0.04) | (0.04) | (0.05) | (0.05) | (0.04) | (0.04) | (0.05) | (0.05) |
| 200 | 40 | 0.781 | 0.779 | 0.776 | 0.814 | 0.784 | 0.776 | 0.787 | 0.813 |
| | | (0.04) | (0.05) | (0.06) | (0.07) | (0.05) | (0.05) | (0.06) | (0.07) |
| | 60 | 0.769 | 0.764 | 0.759 | 0.789 | 0.772 | 0.762 | 0.776 | 0.795 |
| | | (0.05) | (0.05) | (0.05) | (0.06) | (0.05) | (0.05) | (0.05) | (0.06) |
| | 20 | 0.891 | 0.908 | 0.898 | 0.939 | 0.899 | 0.912 | 0.896 | 0.934 |
| | | (0.02) | (0.02) | (0.02) | (0.02) | (0.02) | (0.02) | (0.02) | (0.02) |
| 600 | 40 | 0.891 | 0.898 | 0.892 | 0.923 | 0.899 | 0.902 | 0.897 | 0.925 |
| | | (0.02) | (0.02) | (0.02) | (0.03) | (0.02) | (0.02) | (0.02) | (0.03) |
| | 60 | 0.883 | 0.889 | 0.873 | 0.910 | 0.893 | 0.893 | 0.886 | 0.915 |
| | | (0.03) | (0.02) | (0.02) | (0.03) | (0.02) | (0.03) | (0.02) | (0.02) |
| | 20 | 0.921 | 0.929 | 0.924 | 0.957 | 0.927 | 0.931 | 0.921 | 0.948 |
| | | (0.02) | (0.03) | (0.02) | (0.02) | (0.01) | (0.03) | (0.02) | (0.02) |
| 1000 | 40 | 0.918 | 0.929 | 0.916 | 0.949 | 0.924 | 0.933 | 0.918 | 0.946 |
| | | (0.02) | (0.02) | (0.01) | (0.02) | (0.01) | (0.01) | (0.02) | (0.02) |
| | 60 | 0.915 | 0.923 | 0.908 | 0.942 | 0.921 | 0.926 | 0.915 | 0.942 |
| | | (0.02) | (0.02) | (0.02) | (0.03) | (0.02) | (0.02) | (0.02) | (0.02) |

Table 3: Average accuracy and their standard deviations (in parenthesis) in the randomized trial in Case 2: Linear interaction functions with complicated treatment-free effect functions.

| n | p | $AD$ | $AD_e$ | $AD_s$ | $AD_{e,s}$ | $SABD$ | $SABD_e$ | $SABD_s$ | $SABD_{e,s}$ |
|---|---|---|---|---|---|---|---|---|---|
| | | | | | | Original | | | |
| | 20 | 0.348 | 0.324 | 0.366 | 0.373 | 0.440 | 0.412 | 0.488 | 0.525 |
| | | (0.12) | (0.14) | (0.14) | (0.18) | (0.13) | (0.16) | (0.14) | (0.21) |
| 200 | 40 | 0.316 | 0.295 | 0.346 | 0.329 | 0.379 | 0.337 | 0.454 | 0.457 |
| | | (0.10) | (0.10) | (0.13) | (0.14) | (0.12) | (0.12) | (0.15) | (0.19) |
| | 60 | 0.309 | 0.286 | 0.336 | 0.310 | 0.345 | 0.320 | 0.454 | 0.444 |
| | | (0.08) | (0.08) | (0.11) | (0.12) | (0.10) | (0.10) | (0.13) | (0.18) |
| | 20 | 0.440 | 0.472 | 0.470 | 0.523 | 0.693 | 0.773 | 0.695 | 0.785 |
| | | (0.15) | (0.19) | (0.17) | (0.22) | (0.08) | (0.12) | (0.09) | (0.12) |
| 600 | 40 | 0.391 | 0.415 | 0.409 | 0.448 | 0.660 | 0.736 | 0.692 | 0.784 |
| | | (0.16) | (0.20) | (0.17) | (0.21) | (0.11) | (0.13) | (0.10) | (0.12) |
| | 60 | 0.367 | 0.367 | 0.387 | 0.408 | 0.599 | 0.688 | 0.669 | 0.757 |
| | | (0.14) | (0.16) | (0.15) | (0.18) | (0.13) | (0.15) | (0.11) | (0.12) |
| | 20 | 0.458 | 0.473 | 0.477 | 0.528 | 0.769 | 0.818 | 0.746 | 0.816 |
| | | (0.16) | (0.18) | (0.17) | (0.21) | (0.07) | (0.10) | (0.08) | (0.08) |
| 1000 | 40 | 0.408 | 0.410 | 0.436 | 0.480 | 0.741 | 0.826 | 0.744 | 0.827 |
| | | (0.17) | (0.19) | (0.18) | (0.21) | (0.09) | (0.09) | (0.09) | (0.09) |
| | 60 | 0.436 | 0.467 | 0.465 | 0.492 | 0.746 | 0.833 | 0.766 | 0.816 |
| | | (0.17) | (0.20) | (0.18) | (0.21) | (0.09) | (0.10) | (0.06) | (0.12) |
| | | | | | | Augmented | | | |
| | 20 | 0.417 | 0.418 | 0.436 | 0.474 | 0.650 | 0.647 | 0.716 | 0.740 |
| | | (0.18) | (0.18) | (0.19) | (0.20) | (0.10) | (0.11) | (0.10) | (0.10) |
| 200 | 40 | 0.367 | 0.362 | 0.384 | 0.413 | 0.556 | 0.541 | 0.679 | 0.689 |
| | | (0.15) | (0.15) | (0.17) | (0.19) | (0.14) | (0.15) | (0.11) | (0.12) |
| | 60 | 0.353 | 0.348 | 0.374 | 0.393 | 0.517 | 0.506 | 0.648 | 0.665 |
| | | (0.14) | (0.13) | (0.15) | (0.16) | (0.15) | (0.14) | (0.11) | (0.12) |
| | 20 | 0.492 | 0.520 | 0.514 | 0.583 | 0.857 | 0.862 | 0.873 | 0.894 |
| | | (0.19) | (0.20) | (0.20) | (0.21) | (0.04) | (0.04) | (0.03) | (0.06) |
| 600 | 40 | 0.437 | 0.444 | 0.457 | 0.481 | 0.833 | 0.835 | 0.864 | 0.878 |
| | | (0.20) | (0.20) | (0.20) | (0.22) | (0.05) | (0.04) | (0.04) | (0.04) |
| | 60 | 0.396 | 0.399 | 0.409 | 0.444 | 0.803 | 0.798 | 0.843 | 0.861 |
| | | (0.17) | (0.18) | (0.17) | (0.19) | (0.06) | (0.08) | (0.04) | (0.04) |
| | 20 | 0.485 | 0.492 | 0.501 | 0.541 | 0.897 | 0.896 | 0.905 | 0.918 |
| | | (0.18) | (0.19) | (0.19) | (0.21) | (0.03) | (0.04) | (0.02) | (0.04) |
| 1000 | 40 | 0.433 | 0.436 | 0.453 | 0.504 | 0.888 | 0.890 | 0.903 | 0.921 |
| | | (0.19) | (0.20) | (0.20) | (0.21) | (0.03) | (0.03) | (0.02) | (0.03) |
| | 60 | 0.469 | 0.485 | 0.490 | 0.513 | 0.884 | 0.883 | 0.899 | 0.908 |
| | | (0.19) | (0.20) | (0.20) | (0.21) | (0.03) | (0.03) | (0.02) | (0.03) |

Table 4: Average accuracy and their standard deviations (in parenthesis) in the randomized trial in Case 3: Non-linear interaction functions with simple treatment-free effect functions.

| n | p | $AD$ | $AD_e$ | $AD_s$ | $AD_{e,s}$ | $SABD$ | $SABD_e$ | $SABD_s$ | $SABD_{e,s}$ |
|---|---|---|---|---|---|---|---|---|---|
| | | | | | | Original | | | |
| | 20 | 0.385 | 0.353 | 0.411 | 0.404 | 0.476 | 0.438 | 0.521 | 0.550 |
| | | (0.12) | (0.13) | (0.12) | (0.17) | (0.12) | (0.14) | (0.12) | (0.16) |
| 200 | 40 | 0.339 | 0.316 | 0.375 | 0.358 | 0.408 | 0.366 | 0.473 | 0.486 |
| | | (0.12) | (0.11) | (0.11) | (0.14) | (0.12) | (0.12) | (0.10) | (0.14) |
| | 60 | 0.319 | 0.300 | 0.357 | 0.342 | 0.370 | 0.343 | 0.454 | 0.438 |
| | | (0.11) | (0.10) | (0.11) | (0.12) | (0.12) | (0.11) | (0.10) | (0.13) |
| | 20 | 0.534 | 0.561 | 0.558 | 0.630 | 0.678 | 0.705 | 0.659 | 0.746 |
| | | (0.11) | (0.11) | (0.10) | (0.10) | (0.07) | (0.07) | (0.08) | (0.07) |
| 600 | 40 | 0.490 | 0.486 | 0.529 | 0.576 | 0.651 | 0.653 | 0.651 | 0.723 |
| | | (0.11) | (0.13) | (0.08) | (0.11) | (0.07) | (0.09) | (0.07) | (0.08) |
| | 60 | 0.472 | 0.460 | 0.501 | 0.536 | 0.637 | 0.648 | 0.639 | 0.687 |
| | | (0.10) | (0.12) | (0.08) | (0.10) | (0.07) | (0.07) | (0.07) | (0.07) |
| | 20 | 0.577 | 0.598 | 0.599 | 0.690 | 0.740 | 0.755 | 0.689 | 0.768 |
| | | (0.09) | (0.09) | (0.08) | (0.06) | (0.06) | (0.05) | (0.08) | (0.07) |
| 1000 | 40 | 0.584 | 0.583 | 0.599 | 0.663 | 0.724 | 0.728 | 0.702 | 0.779 |
| | | (0.10) | (0.09) | (0.09) | (0.07) | (0.06) | (0.08) | (0.06) | (0.06) |
| | 60 | 0.574 | 0.579 | 0.584 | 0.636 | 0.723 | 0.731 | 0.707 | 0.760 |
| | | (0.09) | (0.09) | (0.06) | (0.08) | (0.05) | (0.05) | (0.05) | (0.07) |
| | | | | | | Augmented | | | |
| | 20 | 0.524 | 0.524 | 0.555 | 0.603 | 0.608 | 0.604 | 0.642 | 0.687 |
| | | (0.11) | (0.12) | (0.10) | (0.11) | (0.08) | (0.09) | (0.07) | (0.08) |
| 200 | 40 | 0.476 | 0.463 | 0.513 | 0.548 | 0.565 | 0.545 | 0.607 | 0.630 |
| | | (0.12) | (0.13) | (0.10) | (0.11) | (0.10) | (0.09) | (0.07) | (0.08) |
| | 60 | 0.443 | 0.431 | 0.484 | 0.508 | 0.525 | 0.505 | 0.601 | 0.601 |
| | | (0.12) | (0.13) | (0.12) | (0.13) | (0.11) | (0.12) | (0.08) | (0.09) |
| | 20 | 0.674 | 0.680 | 0.685 | 0.719 | 0.776 | 0.784 | 0.776 | 0.805 |
| | | (0.06) | (0.07) | (0.05) | (0.06) | (0.04) | (0.04) | (0.04) | (0.04) |
| 600 | 40 | 0.654 | 0.659 | 0.664 | 0.697 | 0.761 | 0.763 | 0.768 | 0.796 |
| | | (0.07) | (0.06) | (0.06) | (0.06) | (0.04) | (0.04) | (0.04) | (0.04) |
| | 60 | 0.643 | 0.645 | 0.652 | 0.676 | 0.750 | 0.749 | 0.756 | 0.780 |
| | | (0.06) | (0.07) | (0.05) | (0.06) | (0.05) | (0.05) | (0.04) | (0.04) |
| | 20 | 0.706 | 0.708 | 0.715 | 0.746 | 0.805 | 0.805 | 0.787 | 0.817 |
| | | (0.05) | (0.06) | (0.04) | (0.04) | (0.03) | (0.05) | (0.04) | (0.04) |
| 1000 | 40 | 0.704 | 0.704 | 0.711 | 0.738 | 0.805 | 0.809 | 0.807 | 0.831 |
| | | (0.06) | (0.05) | (0.05) | (0.04) | (0.03) | (0.03) | (0.04) | (0.03) |
| | 60 | 0.705 | 0.707 | 0.710 | 0.731 | 0.800 | 0.798 | 0.803 | 0.820 |
| | | (0.05) | (0.04) | (0.04) | (0.04) | (0.03) | (0.03) | (0.04) | (0.04) |

Table 5: Average accuracy and their standard deviations (in parenthesis) in the randomized trial in Case 4: Non-linear interaction functions with a complicated treatment-free effect function.

| n | p | $AD$ | $AD_e$ | $AD_s$ | $AD_{e,s}$ | $SABD$ | $SABD_e$ | $SABD_s$ | $SABD_{e,s}$ |
|---|---|---|---|---|---|---|---|---|---|
| | | | | | | Original | | | |
| | 20 | 0.298 | 0.300 | 0.319 | 0.299 | 0.351 | 0.345 | 0.384 | 0.379 |
| | | (0.09) | (0.09) | (0.10) | (0.10) | (0.12) | (0.12) | (0.13) | (0.16) |
| 200 | 40 | 0.273 | 0.269 | 0.283 | 0.279 | 0.303 | 0.293 | 0.350 | 0.343 |
| | | (0.06) | (0.05) | (0.07) | (0.07) | (0.09) | (0.09) | (0.11) | (0.13) |
| | 60 | 0.279 | 0.275 | 0.299 | 0.289 | 0.298 | 0.287 | 0.359 | 0.340 |
| | | (0.07) | (0.06) | (0.08) | (0.09) | (0.09) | (0.09) | (0.12) | (0.13) |
| | 20 | 0.320 | 0.331 | 0.339 | 0.353 | 0.570 | 0.570 | 0.572 | 0.622 |
| | | (0.11) | (0.12) | (0.12) | (0.13) | (0.12) | (0.13) | (0.13) | (0.13) |
| 600 | 40 | 0.308 | 0.306 | 0.329 | 0.332 | 0.501 | 0.511 | 0.552 | 0.600 |
| | | (0.10) | (0.09) | (0.11) | (0.13) | (0.13) | (0.13) | (0.10) | (0.11) |
| | 60 | 0.297 | 0.293 | 0.308 | 0.307 | 0.463 | 0.473 | 0.529 | 0.578 |
| | | (0.08) | (0.08) | (0.09) | (0.10) | (0.13) | (0.14) | (0.11) | (0.14) |
| | 20 | 0.325 | 0.331 | 0.337 | 0.356 | 0.638 | 0.656 | 0.614 | 0.658 |
| | | (0.11) | (0.12) | (0.12) | (0.14) | (0.09) | (0.09) | (0.10) | (0.10) |
| 1000 | 40 | 0.312 | 0.316 | 0.322 | 0.335 | 0.631 | 0.635 | 0.625 | 0.674 |
| | | (0.10) | (0.11) | (0.11) | (0.13) | (0.09) | (0.12) | (0.09) | (0.09) |
| | 60 | 0.308 | 0.328 | 0.323 | 0.341 | 0.629 | 0.637 | 0.628 | 0.675 |
| | | (0.09) | (0.12) | (0.10) | (0.13) | (0.08) | (0.10) | (0.08) | (0.10) |
| | | | | | | Augmented | | | |
| | 20 | 0.347 | 0.355 | 0.355 | 0.380 | 0.500 | 0.481 | 0.558 | 0.595 |
| | | (0.12) | (0.13) | (0.14) | (0.14) | (0.13) | (0.14) | (0.12) | (0.13) |
| 200 | 40 | 0.288 | 0.287 | 0.299 | 0.314 | 0.376 | 0.377 | 0.506 | 0.499 |
| | | (0.08) | (0.08) | (0.10) | (0.11) | (0.13) | (0.12) | (0.11) | (0.14) |
| | 60 | 0.307 | 0.302 | 0.319 | 0.327 | 0.379 | 0.362 | 0.492 | 0.481 |
| | | (0.10) | (0.09) | (0.11) | (0.12) | (0.13) | (0.12) | (0.13) | (0.14) |
| | 20 | 0.348 | 0.369 | 0.361 | 0.401 | 0.725 | 0.724 | 0.749 | 0.764 |
| | | (0.13) | (0.14) | (0.14) | (0.16) | (0.08) | (0.08) | (0.06) | (0.09) |
| 600 | 40 | 0.349 | 0.353 | 0.360 | 0.382 | 0.667 | 0.653 | 0.729 | 0.747 |
| | | (0.13) | (0.14) | (0.14) | (0.15) | (0.09) | (0.10) | (0.06) | (0.06) |
| | 60 | 0.318 | 0.318 | 0.336 | 0.343 | 0.651 | 0.644 | 0.720 | 0.734 |
| | | (0.12) | (0.11) | (0.13) | (0.13) | (0.10) | (0.11) | (0.07) | (0.07) |
| | 20 | 0.339 | 0.352 | 0.350 | 0.390 | 0.775 | 0.779 | 0.781 | 0.800 |
| | | (0.14) | (0.15) | (0.14) | (0.16) | (0.05) | (0.06) | (0.06) | (0.05) |
| 1000 | 40 | 0.332 | 0.339 | 0.346 | 0.361 | 0.764 | 0.756 | 0.793 | 0.803 |
| | | (0.12) | (0.14) | (0.13) | (0.14) | (0.06) | (0.07) | (0.05) | (0.05) |
| | 60 | 0.331 | 0.347 | 0.342 | 0.358 | 0.766 | 0.753 | 0.788 | 0.803 |
| | | (0.13) | (0.13) | (0.14) | (0.14) | (0.05) | (0.05) | (0.04) | (0.04) |

Table 6: Average accuracy and their standard deviations (in parenthesis) in the observational study in Case 1: Linear interaction functions with simple treatment-free effect functions.

| n | p | $AD$ | $AD_e$ | $AD_s$ | $AD_{e,s}$ | $SABD$ | $SABD_e$ | $SABD_s$ | $SABD_{e,s}$ |
|---|---|---|---|---|---|---|---|---|---|
| | | | | | | Original | | | |
| | 20 | 0.560 | 0.604 | 0.580 | 0.660 | 0.555 | 0.634 | 0.563 | 0.694 |
| | | (0.11) | (0.12) | (0.10) | (0.11) | (0.10) | (0.10) | (0.10) | (0.10) |
| 200 | 40 | 0.494 | 0.499 | 0.532 | 0.589 | 0.517 | 0.550 | 0.542 | 0.624 |
| | | (0.12) | (0.14) | (0.11) | (0.13) | (0.09) | (0.12) | (0.10) | (0.11) |
| | 60 | 0.498 | 0.494 | 0.520 | 0.546 | 0.487 | 0.533 | 0.512 | 0.583 |
| | | (0.11) | (0.13) | (0.09) | (0.13) | (0.11) | (0.12) | (0.10) | (0.12) |
| | 20 | 0.728 | 0.785 | 0.738 | 0.849 | 0.675 | 0.824 | 0.664 | 0.867 |
| | | (0.07) | (0.07) | (0.07) | (0.06) | (0.06) | (0.04) | (0.05) | (0.05) |
| 600 | 40 | 0.716 | 0.754 | 0.721 | 0.824 | 0.674 | 0.802 | 0.670 | 0.856 |
| | | (0.08) | (0.07) | (0.07) | (0.07) | (0.07) | (0.06) | (0.07) | (0.05) |
| | 60 | 0.704 | 0.756 | 0.693 | 0.795 | 0.661 | 0.778 | 0.653 | 0.829 |
| | | (0.08) | (0.07) | (0.07) | (0.08) | (0.08) | (0.06) | (0.07) | (0.05) |
| | 20 | 0.782 | 0.845 | 0.788 | 0.894 | 0.708 | 0.873 | 0.689 | 0.905 |
| | | (0.06) | (0.05) | (0.06) | (0.05) | (0.06) | (0.03) | (0.05) | (0.04) |
| 1000 | 40 | 0.774 | 0.831 | 0.767 | 0.886 | 0.700 | 0.859 | 0.684 | 0.901 |
| | | (0.06) | (0.04) | (0.06) | (0.05) | (0.06) | (0.04) | (0.05) | (0.04) |
| | 60 | 0.764 | 0.825 | 0.755 | 0.868 | 0.702 | 0.845 | 0.684 | 0.885 |
| | | (0.06) | (0.05) | (0.06) | (0.06) | (0.07) | (0.04) | (0.05) | (0.04) |
| | | | | | | Augmented | | | |
| | 20 | 0.692 | 0.699 | 0.690 | 0.727 | 0.712 | 0.722 | 0.702 | 0.741 |
| | | (0.07) | (0.07) | (0.07) | (0.08) | (0.07) | (0.07) | (0.08) | (0.08) |
| 200 | 40 | 0.660 | 0.662 | 0.661 | 0.687 | 0.676 | 0.678 | 0.677 | 0.707 |
| | | (0.09) | (0.08) | (0.08) | (0.09) | (0.07) | (0.08) | (0.08) | (0.08) |
| | 60 | 0.648 | 0.632 | 0.631 | 0.650 | 0.660 | 0.657 | 0.649 | 0.672 |
| | | (0.08) | (0.09) | (0.08) | (0.08) | (0.08) | (0.08) | (0.10) | (0.10) |
| | 20 | 0.840 | 0.861 | 0.841 | 0.889 | 0.841 | 0.881 | 0.836 | 0.904 |
| | | (0.04) | (0.03) | (0.04) | (0.04) | (0.04) | (0.03) | (0.04) | (0.04) |
| 600 | 40 | 0.829 | 0.841 | 0.828 | 0.872 | 0.831 | 0.870 | 0.832 | 0.898 |
| | | (0.05) | (0.04) | (0.05) | (0.04) | (0.05) | (0.03) | (0.04) | (0.03) |
| | 60 | 0.817 | 0.829 | 0.811 | 0.849 | 0.827 | 0.856 | 0.819 | 0.874 |
| | | (0.05) | (0.05) | (0.04) | (0.05) | (0.05) | (0.04) | (0.04) | (0.04) |
| | 20 | 0.879 | 0.900 | 0.879 | 0.926 | 0.867 | 0.919 | 0.861 | 0.932 |
| | | (0.03) | (0.03) | (0.03) | (0.02) | (0.03) | (0.02) | (0.03) | (0.02) |
| 1000 | 40 | 0.878 | 0.892 | 0.874 | 0.917 | 0.867 | 0.909 | 0.860 | 0.927 |
| | | (0.04) | (0.03) | (0.04) | (0.03) | (0.03) | (0.03) | (0.03) | (0.02) |
| | 60 | 0.868 | 0.883 | 0.862 | 0.905 | 0.859 | 0.904 | 0.852 | 0.919 |
| | | (0.03) | (0.03) | (0.03) | (0.04) | (0.04) | (0.02) | (0.04) | (0.03) |

Table 7: Average accuracy and their standard deviations (in parenthesis) in the observational study in Case 2: Linear interaction functions with complicated treatment-free effect functions.

| n | p | $AD$ | $AD_e$ | $AD_s$ | $AD_{e,s}$ | $SABD$ | $SABD_e$ | $SABD_s$ | $SABD_{e,s}$ |
|---|---|---|---|---|---|---|---|---|---|
| | | | | | | Original | | | |
| | 20 | 0.321 | 0.327 | 0.328 | 0.356 | 0.292 | 0.361 | 0.319 | 0.394 |
| | | (0.15) | (0.14) | (0.15) | (0.16) | (0.12) | (0.17) | (0.14) | (0.20) |
| 200 | 40 | 0.301 | 0.300 | 0.315 | 0.325 | 0.284 | 0.318 | 0.309 | 0.345 |
| | | (0.11) | (0.12) | (0.12) | (0.12) | (0.11) | (0.10) | (0.13) | (0.14) |
| | 60 | 0.303 | 0.320 | 0.315 | 0.323 | 0.271 | 0.341 | 0.314 | 0.381 |
| | | (0.11) | (0.12) | (0.11) | (0.12) | (0.12) | (0.13) | (0.12) | (0.15) |
| | 20 | 0.364 | 0.383 | 0.373 | 0.413 | 0.403 | 0.659 | 0.414 | 0.689 |
| | | (0.15) | (0.17) | (0.15) | (0.18) | (0.09) | (0.15) | (0.09) | (0.16) |
| 600 | 40 | 0.355 | 0.379 | 0.368 | 0.388 | 0.360 | 0.571 | 0.404 | 0.662 |
| | | (0.13) | (0.15) | (0.14) | (0.15) | (0.10) | (0.19) | (0.09) | (0.16) |
| | 60 | 0.327 | 0.335 | 0.329 | 0.350 | 0.366 | 0.515 | 0.401 | 0.645 |
| | | (0.11) | (0.14) | (0.12) | (0.14) | (0.10) | (0.19) | (0.09) | (0.17) |
| | 20 | 0.348 | 0.390 | 0.352 | 0.393 | 0.427 | 0.751 | 0.417 | 0.726 |
| | | (0.14) | (0.16) | (0.14) | (0.17) | (0.06) | (0.14) | (0.07) | (0.14) |
| 1000 | 40 | 0.364 | 0.382 | 0.374 | 0.420 | 0.426 | 0.712 | 0.430 | 0.767 |
| | | (0.15) | (0.16) | (0.15) | (0.17) | (0.07) | (0.13) | (0.07) | (0.11) |
| | 60 | 0.372 | 0.415 | 0.386 | 0.449 | 0.413 | 0.741 | 0.436 | 0.771 |
| | | (0.14) | (0.18) | (0.15) | (0.19) | (0.08) | (0.13) | (0.08) | (0.12) |
| | | | | | | Augmented | | | |
| | 20 | 0.368 | 0.393 | 0.382 | 0.440 | 0.576 | 0.590 | 0.636 | 0.662 |
| | | (0.17) | (0.18) | (0.18) | (0.20) | (0.15) | (0.15) | (0.14) | (0.14) |
| 200 | 40 | 0.340 | 0.358 | 0.353 | 0.395 | 0.472 | 0.493 | 0.586 | 0.625 |
| | | (0.15) | (0.15) | (0.16) | (0.18) | (0.17) | (0.16) | (0.14) | (0.15) |
| | 60 | 0.330 | 0.355 | 0.339 | 0.378 | 0.453 | 0.447 | 0.567 | 0.579 |
| | | (0.12) | (0.14) | (0.12) | (0.14) | (0.13) | (0.14) | (0.14) | (0.14) |
| | 20 | 0.403 | 0.446 | 0.414 | 0.497 | 0.809 | 0.827 | 0.828 | 0.861 |
| | | (0.17) | (0.18) | (0.19) | (0.22) | (0.06) | (0.05) | (0.05) | (0.05) |
| 600 | 40 | 0.381 | 0.422 | 0.397 | 0.458 | 0.761 | 0.766 | 0.818 | 0.843 |
| | | (0.16) | (0.18) | (0.17) | (0.18) | (0.09) | (0.08) | (0.05) | (0.06) |
| | 60 | 0.341 | 0.366 | 0.351 | 0.395 | 0.740 | 0.739 | 0.794 | 0.815 |
| | | (0.12) | (0.14) | (0.13) | (0.16) | (0.10) | (0.10) | (0.06) | (0.06) |
| | 20 | 0.374 | 0.420 | 0.381 | 0.449 | 0.853 | 0.862 | 0.864 | 0.892 |
| | | (0.17) | (0.18) | (0.18) | (0.20) | (0.07) | (0.05) | (0.04) | (0.03) |
| 1000 | 40 | 0.392 | 0.429 | 0.403 | 0.443 | 0.836 | 0.848 | 0.867 | 0.885 |
| | | (0.17) | (0.18) | (0.18) | (0.18) | (0.07) | (0.04) | (0.03) | (0.03) |
| | 60 | 0.405 | 0.446 | 0.420 | 0.484 | 0.839 | 0.844 | 0.864 | 0.885 |
| | | (0.17) | (0.19) | (0.18) | (0.20) | (0.04) | (0.05) | (0.03) | (0.03) |

Table 8: Average accuracy and their standard deviations (in parenthesis) in the observational study in Case 3: Non-linear interaction functions with simple treatment-free effect functions.

| n | p | $AD$ | $AD_e$ | $AD_s$ | $AD_{e,s}$ | $SABD$ | $SABD_e$ | $SABD_s$ | $SABD_{e,s}$ |
|---|---|---|---|---|---|---|---|---|---|
| | | | | | | Original | | | |
| | 20 | 0.293 | 0.301 | 0.326 | 0.336 | 0.382 | 0.393 | 0.423 | 0.474 |
| | | (0.08) | (0.09) | (0.10) | (0.14) | (0.10) | (0.12) | (0.10) | (0.17) |
| 200 | 40 | 0.304 | 0.314 | 0.358 | 0.346 | 0.368 | 0.376 | 0.428 | 0.465 |
| | | (0.09) | (0.09) | (0.10) | (0.13) | (0.11) | (0.12) | (0.10) | (0.15) |
| | 60 | 0.277 | 0.284 | 0.330 | 0.345 | 0.337 | 0.331 | 0.401 | 0.416 |
| | | (0.08) | (0.08) | (0.10) | (0.11) | (0.10) | (0.10) | (0.09) | (0.12) |
| | 20 | 0.405 | 0.491 | 0.458 | 0.587 | 0.547 | 0.685 | 0.529 | 0.747 |
| | | (0.11) | (0.12) | (0.11) | (0.13) | (0.07) | (0.07) | (0.06) | (0.08) |
| 600 | 40 | 0.382 | 0.462 | 0.447 | 0.549 | 0.523 | 0.639 | 0.531 | 0.729 |
| | | (0.10) | (0.12) | (0.10) | (0.13) | (0.08) | (0.08) | (0.07) | (0.08) |
| | 60 | 0.386 | 0.438 | 0.465 | 0.545 | 0.519 | 0.628 | 0.534 | 0.696 |
| | | (0.10) | (0.11) | (0.08) | (0.11) | (0.08) | (0.09) | (0.06) | (0.08) |
| | 20 | 0.471 | 0.572 | 0.518 | 0.671 | 0.587 | 0.739 | 0.548 | 0.779 |
| | | (0.09) | (0.10) | (0.08) | (0.08) | (0.06) | (0.05) | (0.06) | (0.06) |
| 1000 | 40 | 0.440 | 0.542 | 0.502 | 0.633 | 0.575 | 0.725 | 0.554 | 0.775 |
| | | (0.08) | (0.09) | (0.07) | (0.08) | (0.06) | (0.05) | (0.05) | (0.07) |
| | 60 | 0.419 | 0.531 | 0.495 | 0.610 | 0.570 | 0.719 | 0.559 | 0.759 |
| | | (0.09) | (0.09) | (0.07) | (0.10) | (0.06) | (0.05) | (0.05) | (0.06) |
| | | | | | | Augmented | | | |
| | 20 | 0.448 | 0.465 | 0.492 | 0.554 | 0.535 | 0.568 | 0.571 | 0.663 |
| | | (0.12) | (0.13) | (0.11) | (0.12) | (0.10) | (0.11) | (0.08) | (0.09) |
| 200 | 40 | 0.456 | 0.461 | 0.505 | 0.544 | 0.511 | 0.534 | 0.562 | 0.629 |
| | | (0.11) | (0.12) | (0.10) | (0.11) | (0.11) | (0.11) | (0.09) | (0.09) |
| | 60 | 0.394 | 0.394 | 0.445 | 0.476 | 0.449 | 0.458 | 0.532 | 0.576 |
| | | (0.11) | (0.12) | (0.11) | (0.11) | (0.11) | (0.12) | (0.09) | (0.09) |
| | 20 | 0.617 | 0.661 | 0.639 | 0.707 | 0.700 | 0.773 | 0.675 | 0.798 |
| | | (0.07) | (0.07) | (0.07) | (0.06) | (0.06) | (0.04) | (0.06) | (0.05) |
| 600 | 40 | 0.606 | 0.643 | 0.631 | 0.688 | 0.691 | 0.753 | 0.685 | 0.792 |
| | | (0.07) | (0.07) | (0.06) | (0.06) | (0.07) | (0.05) | (0.06) | (0.05) |
| | 60 | 0.612 | 0.639 | 0.634 | 0.678 | 0.692 | 0.748 | 0.693 | 0.776 |
| | | (0.07) | (0.07) | (0.07) | (0.08) | (0.06) | (0.04) | (0.06) | (0.05) |
| | 20 | 0.672 | 0.707 | 0.686 | 0.743 | 0.743 | 0.811 | 0.708 | 0.820 |
| | | (0.05) | (0.05) | (0.05) | (0.05) | (0.06) | (0.03) | (0.05) | (0.04) |
| 1000 | 40 | 0.657 | 0.693 | 0.674 | 0.719 | 0.744 | 0.797 | 0.726 | 0.826 |
| | | (0.05) | (0.05) | (0.05) | (0.07) | (0.06) | (0.03) | (0.05) | (0.04) |
| | 60 | 0.648 | 0.687 | 0.662 | 0.714 | 0.724 | 0.794 | 0.710 | 0.815 |
| | | (0.05) | (0.05) | (0.05) | (0.05) | (0.06) | (0.04) | (0.05) | (0.04) |

Table 9: Average accuracy and their standard deviations (in parenthesis) in the observational study in Case 4: Non-linear interaction functions with a complicated treatment-free effect function.

| n | p | $AD$ | $AD_e$ | $AD_s$ | $AD_{e,s}$ | $SABD$ | $SABD_e$ | $SABD_s$ | $SABD_{e,s}$ |
|---|---|---|---|---|---|---|---|---|---|
| | | | | | | Original | | | |
| | 20 | 0.274 | 0.279 | 0.281 | 0.283 | 0.302 | 0.322 | 0.325 | 0.368 |
| | | (0.07) | (0.08) | (0.08) | (0.09) | (0.08) | (0.11) | (0.09) | (0.13) |
| 200 | 40 | 0.270 | 0.270 | 0.285 | 0.276 | 0.285 | 0.305 | 0.322 | 0.353 |
| | | (0.07) | (0.07) | (0.07) | (0.07) | (0.07) | (0.09) | (0.09) | (0.13) |
| | 60 | 0.266 | 0.270 | 0.281 | 0.272 | 0.274 | 0.281 | 0.313 | 0.323 |
| | | (0.06) | (0.06) | (0.07) | (0.06) | (0.07) | (0.08) | (0.08) | (0.10) |
| | 20 | 0.294 | 0.309 | 0.304 | 0.338 | 0.409 | 0.555 | 0.422 | 0.619 |
| | | (0.09) | (0.11) | (0.10) | (0.15) | (0.06) | (0.12) | (0.06) | (0.12) |
| 600 | 40 | 0.283 | 0.286 | 0.300 | 0.314 | 0.369 | 0.489 | 0.412 | 0.580 |
| | | (0.08) | (0.09) | (0.09) | (0.12) | (0.07) | (0.13) | (0.07) | (0.14) |
| | 60 | 0.278 | 0.291 | 0.292 | 0.308 | 0.371 | 0.457 | 0.408 | 0.583 |
| | | (0.07) | (0.08) | (0.08) | (0.11) | (0.07) | (0.15) | (0.06) | (0.12) |
| | 20 | 0.288 | 0.314 | 0.296 | 0.324 | 0.441 | 0.643 | 0.445 | 0.663 |
| | | (0.09) | (0.12) | (0.09) | (0.12) | (0.04) | (0.10) | (0.05) | (0.10) |
| 1000 | 40 | 0.307 | 0.314 | 0.327 | 0.347 | 0.431 | 0.619 | 0.441 | 0.693 |
| | | (0.09) | (0.11) | (0.11) | (0.13) | (0.04) | (0.09) | (0.04) | (0.10) |
| | 60 | 0.289 | 0.312 | 0.306 | 0.344 | 0.421 | 0.603 | 0.445 | 0.668 |
| | | (0.07) | (0.10) | (0.09) | (0.13) | (0.05) | (0.09) | (0.04) | (0.09) |
| | | | | | | Augmented | | | |
| | 20 | 0.317 | 0.336 | 0.335 | 0.384 | 0.439 | 0.456 | 0.505 | 0.571 |
| | | (0.11) | (0.13) | (0.12) | (0.15) | (0.14) | (0.14) | (0.12) | (0.13) |
| 200 | 40 | 0.291 | 0.307 | 0.305 | 0.329 | 0.370 | 0.379 | 0.481 | 0.512 |
| | | (0.09) | (0.10) | (0.10) | (0.11) | (0.13) | (0.12) | (0.12) | (0.14) |
| | 60 | 0.287 | 0.289 | 0.302 | 0.328 | 0.333 | 0.329 | 0.446 | 0.461 |
| | | (0.07) | (0.07) | (0.09) | (0.11) | (0.10) | (0.10) | (0.13) | (0.13) |
| | 20 | 0.337 | 0.361 | 0.346 | 0.390 | 0.660 | 0.702 | 0.676 | 0.761 |
| | | (0.12) | (0.13) | (0.13) | (0.15) | (0.08) | (0.07) | (0.07) | (0.07) |
| 600 | 40 | 0.329 | 0.347 | 0.350 | 0.373 | 0.613 | 0.630 | 0.684 | 0.741 |
| | | (0.12) | (0.13) | (0.13) | (0.15) | (0.10) | (0.12) | (0.07) | (0.06) |
| | 60 | 0.307 | 0.322 | 0.326 | 0.347 | 0.612 | 0.635 | 0.679 | 0.726 |
| | | (0.10) | (0.11) | (0.11) | (0.14) | (0.10) | (0.11) | (0.07) | (0.06) |
| | 20 | 0.318 | 0.342 | 0.325 | 0.369 | 0.734 | 0.769 | 0.730 | 0.801 |
| | | (0.11) | (0.14) | (0.12) | (0.15) | (0.08) | (0.08) | (0.06) | (0.06) |
| 1000 | 40 | 0.332 | 0.346 | 0.348 | 0.379 | 0.714 | 0.741 | 0.741 | 0.796 |
| | | (0.12) | (0.14) | (0.13) | (0.15) | (0.09) | (0.07) | (0.06) | (0.05) |
| | 60 | 0.322 | 0.346 | 0.334 | 0.384 | 0.711 | 0.748 | 0.734 | 0.793 |
| | | (0.11) | (0.13) | (0.12) | (0.15) | (0.07) | (0.06) | (0.06) | (0.04) |

### I.3.2 Empirical values

In this subsection, we show the averages and standard deviations of empirical values from the additional simulation studies. The empirical values of the estimated rule $\mathbb{E}[Y(\hat{d}(\mathbf{X}))]$ are computed by utilizing the generated potential outcomes on the test dataset.

We include the empirical value of the optimal rule $\mathbb{E}[Y(d^{\mathrm{opt}}(\mathbf{X}))]$ as well as the values obtained by assigning everyone to each of the four classes $\mathbb{E}[Y(K)]$ for $K = 1, \ldots, 4$ as benchmarks, assuming the higher value of empirical value is better. Again, the first four tables present the results of experiments that mimic randomized trials, and the remaining four tables show the results of observational studies.

Here, we can observe the benefits of the three components that we found from the treatment decision accuracy results. In addition, the benchmarks of the empirical values reveal some intriguing insights. Firstly, the optimal empirical value provides information on how challenging the current scenario is. For instance, in the randomized trial of Case 1 (Table 9), the optimal empirical value is not significantly different from the mean empirical value evaluated by current methods. Especially with augmented outcomes using a sample size of 1000, it yields a value quite similar to the optimal. On the other hand, in the observational study of Case 4 (Table 16), the obtained empirical values show significant differences from the optimal value. Furthermore, the mean empirical value evaluated from AD-Learning with a sample size of 200 is not markedly different from the empirical value obtained by assigning all individuals to treatment 1. This implies that the estimated ITR provides results akin to a "one-size-fits-all" outcome in small sample sizes, and this finding is not irrelevant to the fact that a majority of AD-Learning results in an intercept-only model as the final decision function under conditions of highly non-linear covariate-outcome relationships. In such instances, we can explore alternative methods to further advance our current results such as other penalty types or performance metrics.

Table 10: Average empirical values and their standard deviations (in parenthesis) in the randomized trial in Case 1: Linear interaction functions with simple treatment-free effect functions. The optimal empirical value is 5.160 and the average empirical values when assigning one treatment are 1.751, 1.752, 1.751, and 1.752, respectively.

| n | p | $AD$ | $AD\_e$ | $AD\_s$ | $AD\_e,s$ | $SABD$ | $SABD\_e$ | $SABD\_s$ | $SABD\_e,s$ |
|---|---|---|---|---|---|---|---|---|---|
| | | | | | Original | | | | |
| | 20 | 4.554 | 4.725 | 4.637 | 4.934 | 4.701 | 4.769 | 4.739 | 4.925 |
| | | (0.25) | (0.21) | (0.23) | (0.17) | (0.21) | (0.19) | (0.21) | (0.16) |
| 200 | 40 | 4.381 | 4.505 | 4.465 | 4.778 | 4.591 | 4.616 | 4.626 | 4.798 |
| | | (0.31) | (0.34) | (0.30) | (0.28) | (0.24) | (0.24) | (0.27) | (0.26) |
| | 60 | 4.364 | 4.409 | 4.412 | 4.670 | 4.528 | 4.566 | 4.578 | 4.746 |
| | | (0.29) | (0.36) | (0.25) | (0.27) | (0.21) | (0.26) | (0.20) | (0.21) |
| | 20 | 4.963 | 5.053 | 4.985 | 5.117 | 4.992 | 5.050 | 4.956 | 5.063 |
| | | (0.10) | (0.10) | (0.09) | (0.07) | (0.09) | (0.10) | (0.11) | (0.06) |
| 600 | 40 | 4.964 | 5.049 | 4.950 | 5.071 | 5.013 | 5.059 | 4.972 | 5.056 |
| | | (0.10) | (0.06) | (0.09) | (0.19) | (0.08) | (0.06) | (0.09) | (0.12) |
| | 60 | 4.935 | 5.025 | 4.891 | 5.059 | 5.006 | 5.045 | 4.962 | 5.054 |
| | | (0.11) | (0.07) | (0.10) | (0.11) | (0.09) | (0.07) | (0.09) | (0.07) |
| | 20 | 5.054 | 5.092 | 5.057 | 5.140 | 5.053 | 5.073 | 5.012 | 5.060 |
| | | (0.07) | (0.11) | (0.07) | (0.07) | (0.07) | (0.08) | (0.09) | (0.07) |
| 1000 | 40 | 5.034 | 5.098 | 5.020 | 5.101 | 5.051 | 5.090 | 5.000 | 5.060 |
| | | (0.07) | (0.05) | (0.07) | (0.16) | (0.07) | (0.05) | (0.09) | (0.09) |
| | 60 | 5.046 | 5.096 | 5.013 | 5.112 | 5.070 | 5.095 | 5.021 | 5.072 |
| | | (0.06) | (0.04) | (0.06) | (0.09) | (0.06) | (0.04) | (0.08) | (0.07) |
| | | | | | Augmented | | | | |
| | 20 | 4.928 | 4.940 | 4.936 | 5.022 | 4.951 | 4.942 | 4.964 | 5.024 |
| | | (0.10) | (0.10) | (0.15) | (0.14) | (0.09) | (0.11) | (0.15) | (0.14) |
| 200 | 40 | 4.887 | 4.873 | 4.857 | 4.940 | 4.888 | 4.863 | 4.884 | 4.936 |
| | | (0.12) | (0.13) | (0.22) | (0.20) | (0.12) | (0.13) | (0.21) | (0.22) |
| | 60 | 4.851 | 4.833 | 4.817 | 4.888 | 4.860 | 4.826 | 4.865 | 4.905 |
| | | (0.12) | (0.14) | (0.17) | (0.16) | (0.13) | (0.15) | (0.16) | (0.16) |
| | 20 | 5.092 | 5.109 | 5.100 | 5.136 | 5.101 | 5.114 | 5.097 | 5.133 |
| | | (0.04) | (0.04) | (0.04) | (0.04) | (0.04) | (0.04) | (0.04) | (0.04) |
| 600 | 40 | 5.095 | 5.101 | 5.096 | 5.124 | 5.105 | 5.105 | 5.102 | 5.127 |
| | | (0.05) | (0.04) | (0.04) | (0.06) | (0.04) | (0.04) | (0.04) | (0.05) |
| | 60 | 5.086 | 5.091 | 5.073 | 5.114 | 5.097 | 5.096 | 5.089 | 5.120 |
| | | (0.05) | (0.05) | (0.05) | (0.05) | (0.05) | (0.05) | (0.05) | (0.04) |
| | 20 | 5.125 | 5.127 | 5.127 | 5.148 | 5.130 | 5.131 | 5.125 | 5.143 |
| | | (0.04) | (0.05) | (0.04) | (0.04) | (0.04) | (0.05) | (0.04) | (0.04) |
| 1000 | 40 | 5.121 | 5.129 | 5.118 | 5.140 | 5.126 | 5.132 | 5.120 | 5.139 |
| | | (0.04) | (0.04) | (0.04) | (0.04) | (0.04) | (0.04) | (0.04) | (0.04) |
| | 60 | 5.121 | 5.128 | 5.115 | 5.140 | 5.127 | 5.131 | 5.120 | 5.141 |
| | | (0.04) | (0.04) | (0.04) | (0.04) | (0.04) | (0.04) | (0.04) | (0.04) |

Table 11: Average empirical values and their standard deviations (in parenthesis) in the randomized trial in Case 2: Linear interaction functions with complicated treatment-free effect functions. The optimal empirical value is 10.333 and the average empirical values when assigning one treatment are 6.923, 6.924, 6.923, and 6.924, respectively.

| n | p | $AD$ | $AD\_e$ | $AD\_s$ | $AD\_e, s$ | $SABD$ | $SABD\_e$ | $SABD\_s$ | $SABD\_e, s$ |
|---|---|---|---|---|---|---|---|---|---|
| | | | | | | Original | | | |
| | 20 | 7.704 | 7.474 | 7.820 | 7.764 | 8.360 | 8.073 | 8.677 | 8.736 |
| | | (0.90) | (1.00) | (0.98) | (1.14) | (0.96) | (1.16) | (0.94) | (1.26) |
| 200 | 40 | 7.430 | 7.262 | 7.641 | 7.482 | 7.907 | 7.529 | 8.415 | 8.339 |
| | | (0.80) | (0.77) | (0.94) | (0.98) | (0.93) | (0.92) | (1.08) | (1.19) |
| | 60 | 7.413 | 7.205 | 7.605 | 7.359 | 7.690 | 7.470 | 8.475 | 8.275 |
| | | (0.79) | (0.74) | (0.93) | (0.94) | (0.91) | (0.83) | (0.94) | (1.23) |
| | 20 | 8.334 | 8.437 | 8.488 | 8.719 | 9.791 | 9.966 | 9.803 | 10.023 |
| | | (1.11) | (1.27) | (1.18) | (1.29) | (0.56) | (0.76) | (0.56) | (0.63) |
| 600 | 40 | 7.983 | 8.051 | 8.079 | 8.250 | 9.610 | 9.821 | 9.757 | 9.991 |
| | | (1.11) | (1.25) | (1.16) | (1.33) | (0.69) | (0.73) | (0.64) | (0.69) |
| | 60 | 7.834 | 7.788 | 7.956 | 8.025 | 9.311 | 9.648 | 9.657 | 9.930 |
| | | (1.06) | (1.12) | (1.14) | (1.26) | (0.85) | (0.80) | (0.66) | (0.59) |
| | 20 | 8.395 | 8.458 | 8.486 | 8.706 | 9.997 | 10.087 | 9.929 | 10.101 |
| | | (1.15) | (1.20) | (1.19) | (1.27) | (0.47) | (0.54) | (0.50) | (0.50) |
| 1000 | 40 | 8.033 | 7.993 | 8.199 | 8.400 | 9.874 | 10.077 | 9.893 | 10.101 |
| | | (1.20) | (1.24) | (1.25) | (1.32) | (0.59) | (0.51) | (0.58) | (0.47) |
| | 60 | 8.275 | 8.403 | 8.439 | 8.506 | 9.933 | 10.122 | 10.021 | 10.053 |
| | | (1.17) | (1.24) | (1.21) | (1.32) | (0.54) | (0.59) | (0.44) | (0.65) |
| | | | | | | Augmented | | | |
| | 20 | 8.111 | 8.117 | 8.201 | 8.429 | 9.552 | 9.526 | 9.801 | 9.877 |
| | | (1.19) | (1.16) | (1.24) | (1.28) | (0.64) | (0.65) | (0.53) | (0.49) |
| 200 | 40 | 7.758 | 7.712 | 7.859 | 8.038 | 9.017 | 8.912 | 9.646 | 9.678 |
| | | (1.09) | (1.09) | (1.17) | (1.28) | (0.97) | (1.02) | (0.62) | (0.65) |
| | 60 | 7.694 | 7.649 | 7.834 | 7.939 | 8.814 | 8.764 | 9.519 | 9.589 |
| | | (1.03) | (1.01) | (1.11) | (1.15) | (0.96) | (0.95) | (0.70) | (0.67) |
| | 20 | 8.551 | 8.713 | 8.663 | 9.028 | 10.236 | 10.244 | 10.265 | 10.278 |
| | | (1.25) | (1.23) | (1.28) | (1.21) | (0.36) | (0.35) | (0.35) | (0.39) |
| 600 | 40 | 8.212 | 8.237 | 8.317 | 8.437 | 10.173 | 10.176 | 10.234 | 10.249 |
| | | (1.25) | (1.27) | (1.27) | (1.34) | (0.37) | (0.37) | (0.36) | (0.36) |
| | 60 | 7.972 | 7.998 | 8.048 | 8.249 | 10.100 | 10.080 | 10.201 | 10.230 |
| | | (1.16) | (1.16) | (1.20) | (1.27) | (0.38) | (0.44) | (0.34) | (0.33) |
| | 20 | 8.509 | 8.549 | 8.587 | 8.769 | 10.262 | 10.255 | 10.271 | 10.280 |
| | | (1.22) | (1.22) | (1.24) | (1.26) | (0.41) | (0.41) | (0.41) | (0.41) |
| 1000 | 40 | 8.158 | 8.154 | 8.265 | 8.543 | 10.230 | 10.232 | 10.252 | 10.266 |
| | | (1.26) | (1.27) | (1.31) | (1.35) | (0.38) | (0.38) | (0.38) | (0.38) |
| | 60 | 8.414 | 8.510 | 8.532 | 8.648 | 10.262 | 10.260 | 10.281 | 10.287 |
| | | (1.26) | (1.25) | (1.28) | (1.31) | (0.40) | (0.40) | (0.40) | (0.40) |

Table 12: Average empirical values and their standard deviations (in parenthesis) in the randomized trial in Case 3: Non-linear interaction functions with simple treatment-free effect functions. The optimal empirical value is 1.422 and the average empirical values when assigning one treatment are 0.041, -0.919, -0.419, and -0.920, respectively.

| n | p | $AD$ | $AD\_e$ | $AD\_s$ | $AD\_{e,s}$ | $SABD$ | $SABD\_e$ | $SABD\_s$ | $SABD\_{e,s}$ |
|---|---|---|---|---|---|---|---|---|---|
| | | | | | | Original | | | |
| | 20 | 0.210 | 0.157 | 0.294 | 0.382 | 0.434 | 0.275 | 0.546 | 0.592 |
| | | (0.45) | (0.48) | (0.44) | (0.45) | (0.47) | (0.55) | (0.46) | (0.59) |
| 200 | 40 | 0.023 | -0.018 | 0.131 | 0.185 | 0.196 | 0.044 | 0.436 | 0.462 |
| | | (0.46) | (0.46) | (0.44) | (0.45) | (0.50) | (0.54) | (0.38) | (0.50) |
| | 60 | -0.051 | -0.087 | 0.072 | 0.100 | 0.081 | -0.008 | 0.346 | 0.315 |
| | | (0.49) | (0.45) | (0.45) | (0.45) | (0.51) | (0.48) | (0.43) | (0.48) |
| | 20 | 0.733 | 0.826 | 0.788 | 0.983 | 0.963 | 1.023 | 0.931 | 1.095 |
| | | (0.29) | (0.26) | (0.27) | (0.21) | (0.18) | (0.17) | (0.23) | (0.17) |
| 600 | 40 | 0.631 | 0.646 | 0.715 | 0.857 | 0.910 | 0.915 | 0.923 | 1.025 |
| | | (0.33) | (0.32) | (0.22) | (0.29) | (0.21) | (0.24) | (0.18) | (0.21) |
| | 60 | 0.609 | 0.585 | 0.651 | 0.770 | 0.905 | 0.922 | 0.903 | 0.993 |
| | | (0.28) | (0.32) | (0.22) | (0.27) | (0.19) | (0.18) | (0.17) | (0.17) |
| | 20 | 0.879 | 0.930 | 0.920 | 1.111 | 1.089 | 1.104 | 1.009 | 1.123 |
| | | (0.22) | (0.22) | (0.19) | (0.12) | (0.15) | (0.14) | (0.18) | (0.14) |
| 1000 | 40 | 0.864 | 0.880 | 0.887 | 1.061 | 1.027 | 1.020 | 0.993 | 1.117 |
| | | (0.23) | (0.21) | (0.21) | (0.14) | (0.15) | (0.25) | (0.16) | (0.14) |
| | 60 | 0.870 | 0.890 | 0.878 | 1.002 | 1.050 | 1.064 | 1.030 | 1.102 |
| | | (0.20) | (0.21) | (0.16) | (0.24) | (0.14) | (0.14) | (0.14) | (0.16) |
| | | | | | | Augmented | | | |
| | 20 | 0.715 | 0.697 | 0.794 | 0.918 | 0.883 | 0.871 | 0.967 | 1.056 |
| | | (0.32) | (0.38) | (0.24) | (0.24) | (0.19) | (0.26) | (0.17) | (0.18) |
| 200 | 40 | 0.588 | 0.545 | 0.680 | 0.779 | 0.791 | 0.724 | 0.884 | 0.926 |
| | | (0.30) | (0.36) | (0.27) | (0.25) | (0.26) | (0.30) | (0.20) | (0.22) |
| | 60 | 0.481 | 0.444 | 0.578 | 0.653 | 0.661 | 0.587 | 0.849 | 0.840 |
| | | (0.35) | (0.38) | (0.32) | (0.35) | (0.32) | (0.42) | (0.19) | (0.23) |
| | 20 | 1.082 | 1.093 | 1.100 | 1.151 | 1.184 | 1.194 | 1.216 | 1.245 |
| | | (0.11) | (0.12) | (0.10) | (0.10) | (0.10) | (0.10) | (0.07) | (0.08) |
| 600 | 40 | 1.051 | 1.065 | 1.063 | 1.118 | 1.166 | 1.163 | 1.189 | 1.220 |
| | | (0.12) | (0.11) | (0.11) | (0.12) | (0.11) | (0.10) | (0.10) | (0.09) |
| | 60 | 1.039 | 1.042 | 1.051 | 1.092 | 1.161 | 1.156 | 1.173 | 1.204 |
| | | (0.12) | (0.13) | (0.10) | (0.12) | (0.10) | (0.10) | (0.09) | (0.09) |
| | 20 | 1.141 | 1.134 | 1.156 | 1.196 | 1.219 | 1.210 | 1.229 | 1.260 |
| | | (0.08) | (0.15) | (0.08) | (0.07) | (0.08) | (0.12) | (0.08) | (0.07) |
| 1000 | 40 | 1.133 | 1.137 | 1.143 | 1.183 | 1.207 | 1.213 | 1.231 | 1.258 |
| | | (0.09) | (0.09) | (0.09) | (0.07) | (0.07) | (0.08) | (0.07) | (0.06) |
| | 60 | 1.141 | 1.146 | 1.148 | 1.177 | 1.212 | 1.207 | 1.235 | 1.250 |
| | | (0.09) | (0.08) | (0.08) | (0.08) | (0.09) | (0.08) | (0.08) | (0.08) |

Table 13: Average empirical values and their standard deviations (in parenthesis) in the randomized trial in Case 4: Non-linear interaction functions with a complicated treatment-free effect function. The optimal empirical value is 5.623 and the average empirical values when assigning one treatment are 4.243, 3.282, 3.782, and 3.281 respectively.

| n | p | $AD$ | $AD\_e$ | $AD\_s$ | $AD\_e,s$ | $SABD$ | $SABD\_e$ | $SABD\_s$ | $SABD\_e,s$ |
|---|---|---|---|---|---|---|---|---|---|
| | | | | | | Original | | | |
| | 20 | 3.979 | 4.033 | 4.058 | 4.052 | 4.192 | 4.132 | 4.381 | 4.366 |
| | | (0.55) | (0.53) | (0.57) | (0.58) | (0.58) | (0.61) | (0.58) | (0.65) |
| 200 | 40 | 3.890 | 3.876 | 3.920 | 3.913 | 3.984 | 3.936 | 4.184 | 4.168 |
| | | (0.55) | (0.54) | (0.57) | (0.59) | (0.65) | (0.61) | (0.63) | (0.65) |
| | 60 | 3.939 | 3.931 | 3.991 | 3.970 | 4.018 | 3.938 | 4.241 | 4.137 |
| | | (0.57) | (0.56) | (0.58) | (0.59) | (0.60) | (0.59) | (0.63) | (0.70) |
| | 20 | 4.134 | 4.198 | 4.202 | 4.282 | 4.995 | 4.980 | 5.009 | 5.127 |
| | | (0.60) | (0.64) | (0.64) | (0.62) | (0.49) | (0.52) | (0.46) | (0.48) |
| 600 | 40 | 4.032 | 4.066 | 4.111 | 4.183 | 4.761 | 4.774 | 4.951 | 5.056 |
| | | (0.64) | (0.64) | (0.68) | (0.66) | (0.57) | (0.60) | (0.48) | (0.49) |
| | 60 | 3.989 | 4.021 | 4.029 | 4.042 | 4.667 | 4.685 | 4.885 | 4.964 |
| | | (0.64) | (0.62) | (0.66) | (0.67) | (0.61) | (0.62) | (0.49) | (0.60) |
| | 20 | 4.117 | 4.130 | 4.152 | 4.218 | 5.138 | 5.168 | 5.083 | 5.191 |
| | | (0.66) | (0.67) | (0.68) | (0.71) | (0.43) | (0.44) | (0.46) | (0.42) |
| 1000 | 40 | 4.047 | 4.061 | 4.093 | 4.188 | 5.075 | 5.078 | 5.070 | 5.188 |
| | | (0.62) | (0.64) | (0.66) | (0.71) | (0.49) | (0.51) | (0.48) | (0.43) |
| | 60 | 4.103 | 4.190 | 4.154 | 4.232 | 5.144 | 5.152 | 5.155 | 5.237 |
| | | (0.64) | (0.67) | (0.65) | (0.69) | (0.45) | (0.49) | (0.43) | (0.48) |
| | | | | | | Augmented | | | |
| | 20 | 4.203 | 4.247 | 4.225 | 4.333 | 4.733 | 4.648 | 4.923 | 5.016 |
| | | (0.58) | (0.58) | (0.60) | (0.61) | (0.58) | (0.66) | (0.52) | (0.55) |
| 200 | 40 | 3.962 | 3.962 | 4.010 | 4.029 | 4.332 | 4.288 | 4.745 | 4.700 |
| | | (0.60) | (0.62) | (0.61) | (0.65) | (0.64) | (0.64) | (0.54) | (0.59) |
| | 60 | 4.011 | 3.987 | 4.050 | 4.095 | 4.350 | 4.318 | 4.707 | 4.643 |
| | | (0.64) | (0.61) | (0.67) | (0.64) | (0.61) | (0.55) | (0.56) | (0.63) |
| | 20 | 4.245 | 4.293 | 4.291 | 4.425 | 5.307 | 5.302 | 5.364 | 5.375 |
| | | (0.63) | (0.68) | (0.65) | (0.68) | (0.38) | (0.39) | (0.36) | (0.39) |
| 600 | 40 | 4.199 | 4.262 | 4.227 | 4.328 | 5.202 | 5.160 | 5.320 | 5.334 |
| | | (0.69) | (0.67) | (0.71) | (0.73) | (0.42) | (0.39) | (0.37) | (0.36) |
| | 60 | 4.073 | 4.084 | 4.140 | 4.166 | 5.175 | 5.161 | 5.313 | 5.327 |
| | | (0.71) | (0.71) | (0.74) | (0.74) | (0.41) | (0.39) | (0.34) | (0.35) |
| | 20 | 4.147 | 4.199 | 4.175 | 4.331 | 5.356 | 5.355 | 5.372 | 5.393 |
| | | (0.71) | (0.70) | (0.72) | (0.75) | (0.39) | (0.41) | (0.42) | (0.42) |
| 1000 | 40 | 4.124 | 4.141 | 4.168 | 4.214 | 5.331 | 5.310 | 5.369 | 5.371 |
| | | (0.69) | (0.71) | (0.72) | (0.77) | (0.38) | (0.40) | (0.38) | (0.40) |
| | 60 | 4.183 | 4.264 | 4.213 | 4.298 | 5.377 | 5.362 | 5.418 | 5.433 |
| | | (0.69) | (0.69) | (0.73) | (0.71) | (0.40) | (0.41) | (0.40) | (0.40) |

Table 14: Average empirical values and their standard deviations (in parenthesis) in the observational study in Case 1: Linear interaction functions with simple treatment-free effect functions. The optimal empirical value is 4.263 and the average empirical values when assigning one treatment are 1.503, 2.003, 2.502, and 2.001, respectively.

| n | p | $AD$ | $AD\_e$ | $AD\_s$ | $AD\_e,s$ | $SABD$ | $SABD\_e$ | $SABD\_s$ | $SABD\_e,s$ |
|---|---|---|---|---|---|---|---|---|---|
| | | | | | Original | | | | |
| | 20 | 3.511 | 3.621 | 3.581 | 3.790 | 3.553 | 3.730 | 3.566 | 3.879 |
| | | (0.41) | (0.46) | (0.36) | (0.36) | (0.29) | (0.33) | (0.32) | (0.30) |
| 200 | 40 | 3.258 | 3.223 | 3.415 | 3.558 | 3.430 | 3.450 | 3.503 | 3.704 |
| | | (0.51) | (0.62) | (0.43) | (0.51) | (0.30) | (0.50) | (0.29) | (0.38) |
| | 60 | 3.289 | 3.210 | 3.382 | 3.412 | 3.304 | 3.396 | 3.391 | 3.564 |
| | | (0.47) | (0.58) | (0.38) | (0.53) | (0.44) | (0.51) | (0.38) | (0.47) |
| | 20 | 3.992 | 4.079 | 4.009 | 4.165 | 3.881 | 4.143 | 3.851 | 4.188 |
| | | (0.13) | (0.11) | (0.12) | (0.09) | (0.15) | (0.07) | (0.14) | (0.08) |
| 600 | 40 | 3.970 | 4.027 | 3.982 | 4.135 | 3.888 | 4.115 | 3.874 | 4.182 |
| | | (0.15) | (0.23) | (0.14) | (0.12) | (0.17) | (0.12) | (0.15) | (0.08) |
| | 60 | 3.942 | 4.037 | 3.918 | 4.094 | 3.848 | 4.079 | 3.826 | 4.147 |
| | | (0.16) | (0.12) | (0.15) | (0.13) | (0.18) | (0.11) | (0.16) | (0.08) |
| | 20 | 4.086 | 4.166 | 4.094 | 4.211 | 3.942 | 4.200 | 3.893 | 4.224 |
| | | (0.09) | (0.08) | (0.09) | (0.07) | (0.12) | (0.05) | (0.12) | (0.06) |
| 1000 | 40 | 4.073 | 4.150 | 4.061 | 4.205 | 3.931 | 4.183 | 3.892 | 4.219 |
| | | (0.11) | (0.08) | (0.11) | (0.07) | (0.12) | (0.06) | (0.12) | (0.06) |
| | 60 | 4.069 | 4.150 | 4.050 | 4.190 | 3.941 | 4.177 | 3.900 | 4.212 |
| | | (0.10) | (0.07) | (0.10) | (0.11) | (0.14) | (0.06) | (0.13) | (0.07) |
| | | | | | Augmented | | | | |
| | 20 | 3.913 | 3.922 | 3.894 | 3.970 | 3.964 | 3.969 | 3.923 | 3.993 |
| | | (0.17) | (0.18) | (0.18) | (0.17) | (0.16) | (0.17) | (0.20) | (0.18) |
| 200 | 40 | 3.828 | 3.827 | 3.824 | 3.879 | 3.877 | 3.874 | 3.869 | 3.927 |
| | | (0.22) | (0.23) | (0.21) | (0.25) | (0.17) | (0.20) | (0.21) | (0.21) |
| | 60 | 3.806 | 3.739 | 3.737 | 3.789 | 3.847 | 3.817 | 3.793 | 3.831 |
| | | (0.22) | (0.35) | (0.27) | (0.24) | (0.20) | (0.24) | (0.31) | (0.31) |
| | 20 | 4.169 | 4.188 | 4.168 | 4.210 | 4.172 | 4.207 | 4.165 | 4.222 |
| | | (0.06) | (0.05) | (0.06) | (0.06) | (0.06) | (0.05) | (0.06) | (0.05) |
| 600 | 40 | 4.158 | 4.169 | 4.156 | 4.200 | 4.162 | 4.201 | 4.162 | 4.224 |
| | | (0.07) | (0.06) | (0.07) | (0.06) | (0.06) | (0.05) | (0.06) | (0.06) |
| | 60 | 4.144 | 4.153 | 4.135 | 4.175 | 4.154 | 4.186 | 4.146 | 4.201 |
| | | (0.08) | (0.08) | (0.07) | (0.08) | (0.07) | (0.06) | (0.07) | (0.07) |
| | 20 | 4.208 | 4.223 | 4.207 | 4.239 | 4.197 | 4.235 | 4.191 | 4.242 |
| | | (0.05) | (0.05) | (0.05) | (0.05) | (0.05) | (0.04) | (0.05) | (0.04) |
| 1000 | 40 | 4.205 | 4.215 | 4.201 | 4.232 | 4.196 | 4.228 | 4.188 | 4.238 |
| | | (0.06) | (0.05) | (0.06) | (0.05) | (0.06) | (0.05) | (0.06) | (0.05) |
| | 60 | 4.204 | 4.214 | 4.198 | 4.230 | 4.194 | 4.232 | 4.187 | 4.240 |
| | | (0.05) | (0.05) | (0.05) | (0.05) | (0.05) | (0.04) | (0.06) | (0.04) |

Table 15: Average empirical values and their standard deviations (in parenthesis) in the observational study in Case 2: Linear interaction functions with complicated treatment-free effect functions. The optimal empirical value is 9.433 and the average empirical values when assigning one treatment are 6.672, 7.173, 7.672, and 7.171, respectively.

| n | p | $AD$ | $AD\_e$ | $AD\_s$ | $AD\_e,s$ | $SABD$ | $SABD\_e$ | $SABD\_s$ | $SABD\_e,s$ |
|---|---|---|---|---|---|---|---|---|---|
| | | | | | | Original | | | |
| | 20 | 7.592 | 7.598 | 7.608 | 7.696 | 7.561 | 7.736 | 7.715 | 7.927 |
| | | (0.73) | (0.74) | (0.75) | (0.78) | (0.67) | (0.80) | (0.72) | (0.87) |
| 200 | 40 | 7.462 | 7.452 | 7.540 | 7.554 | 7.462 | 7.497 | 7.647 | 7.671 |
| | | (0.60) | (0.66) | (0.67) | (0.67) | (0.64) | (0.58) | (0.68) | (0.69) |
| | 60 | 7.503 | 7.531 | 7.568 | 7.538 | 7.380 | 7.612 | 7.645 | 7.815 |
| | | (0.58) | (0.61) | (0.58) | (0.65) | (0.68) | (0.66) | (0.63) | (0.76) |
| | 20 | 7.798 | 7.824 | 7.840 | 7.970 | 8.147 | 8.948 | 8.197 | 9.024 |
| | | (0.70) | (0.79) | (0.72) | (0.84) | (0.46) | (0.58) | (0.45) | (0.61) |
| 600 | 40 | 7.732 | 7.778 | 7.787 | 7.841 | 7.954 | 8.610 | 8.142 | 8.923 |
| | | (0.70) | (0.79) | (0.75) | (0.77) | (0.54) | (0.74) | (0.47) | (0.58) |
| | 60 | 7.615 | 7.599 | 7.625 | 7.666 | 7.953 | 8.413 | 8.125 | 8.875 |
| | | (0.66) | (0.71) | (0.68) | (0.73) | (0.59) | (0.85) | (0.50) | (0.70) |
| | 20 | 7.712 | 7.833 | 7.731 | 7.823 | 8.191 | 9.122 | 8.166 | 9.084 |
| | | (0.77) | (0.81) | (0.79) | (0.87) | (0.47) | (0.59) | (0.48) | (0.60) |
| 1000 | 40 | 7.748 | 7.787 | 7.797 | 7.958 | 8.163 | 9.047 | 8.206 | 9.180 |
| | | (0.75) | (0.81) | (0.74) | (0.83) | (0.42) | (0.51) | (0.43) | (0.43) |
| | 60 | 7.873 | 7.967 | 7.931 | 8.113 | 8.151 | 9.143 | 8.248 | 9.211 |
| | | (0.72) | (0.84) | (0.74) | (0.88) | (0.49) | (0.55) | (0.47) | (0.51) |
| | | | | | | Augmented | | | |
| | 20 | 7.770 | 7.887 | 7.838 | 8.042 | 8.667 | 8.711 | 8.859 | 8.936 |
| | | (0.79) | (0.81) | (0.82) | (0.90) | (0.68) | (0.67) | (0.58) | (0.58) |
| 200 | 40 | 7.621 | 7.713 | 7.680 | 7.854 | 8.240 | 8.308 | 8.703 | 8.802 |
| | | (0.78) | (0.77) | (0.81) | (0.88) | (0.79) | (0.78) | (0.65) | (0.68) |
| | 60 | 7.585 | 7.693 | 7.629 | 7.812 | 8.169 | 8.146 | 8.647 | 8.680 |
| | | (0.65) | (0.69) | (0.65) | (0.72) | (0.68) | (0.66) | (0.65) | (0.65) |
| | 20 | 7.920 | 8.102 | 7.956 | 8.263 | 9.315 | 9.340 | 9.343 | 9.380 |
| | | (0.81) | (0.84) | (0.85) | (0.91) | (0.35) | (0.34) | (0.34) | (0.33) |
| 600 | 40 | 7.792 | 7.962 | 7.867 | 8.103 | 9.194 | 9.211 | 9.311 | 9.342 |
| | | (0.80) | (0.85) | (0.80) | (0.86) | (0.45) | (0.41) | (0.34) | (0.34) |
| | 60 | 7.630 | 7.743 | 7.676 | 7.845 | 9.153 | 9.159 | 9.277 | 9.308 |
| | | (0.68) | (0.74) | (0.74) | (0.84) | (0.43) | (0.45) | (0.34) | (0.34) |
| | 20 | 7.756 | 7.949 | 7.781 | 8.043 | 9.318 | 9.341 | 9.347 | 9.371 |
| | | (0.87) | (0.91) | (0.89) | (0.92) | (0.46) | (0.41) | (0.41) | (0.41) |
| 1000 | 40 | 7.847 | 7.997 | 7.885 | 8.034 | 9.292 | 9.318 | 9.342 | 9.359 |
| | | (0.87) | (0.85) | (0.89) | (0.87) | (0.41) | (0.38) | (0.39) | (0.38) |
| | 60 | 7.948 | 8.076 | 8.006 | 8.246 | 9.343 | 9.346 | 9.373 | 9.393 |
| | | (0.80) | (0.85) | (0.84) | (0.85) | (0.41) | (0.42) | (0.40) | (0.41) |

Table 16: Average empirical values and their standard deviations (in parenthesis) in the observational study in Case 3: Non-linear interaction functions with simple treatment-free effect functions. The optimal empirical value is 1.422 and the average empirical values when assigning one treatment are 0.042, -0.920, -0.419, and -0.920, respectively.

| n | p | $AD$ | $AD\_e$ | $AD\_s$ | $AD\_e,s$ | $SABD$ | $SABD\_e$ | $SABD\_s$ | $SABD\_e,s$ |
|---|---|---|---|---|---|---|---|---|---|
| | | | | | | Original | | | |
| | 20 | 0.029 | -0.014 | 0.113 | 0.142 | 0.312 | 0.178 | 0.405 | 0.411 |
| | | (0.36) | (0.42) | (0.41) | (0.46) | (0.35) | (0.49) | (0.38) | (0.58) |
| 200 | 40 | 0.091 | 0.058 | 0.244 | 0.151 | 0.258 | 0.149 | 0.415 | 0.335 |
| | | (0.33) | (0.38) | (0.28) | (0.42) | (0.40) | (0.53) | (0.36) | (0.59) |
| | 60 | 0.014 | 0.010 | 0.110 | 0.129 | 0.168 | 0.031 | 0.349 | 0.224 |
| | | (0.32) | (0.32) | (0.37) | (0.38) | (0.37) | (0.45) | (0.34) | (0.56) |
| | 20 | 0.482 | 0.660 | 0.607 | 0.910 | 0.773 | 1.008 | 0.721 | 1.095 |
| | | (0.29) | (0.36) | (0.27) | (0.28) | (0.16) | (0.14) | (0.15) | (0.16) |
| 600 | 40 | 0.420 | 0.592 | 0.567 | 0.802 | 0.717 | 0.927 | 0.720 | 1.076 |
| | | (0.25) | (0.28) | (0.23) | (0.29) | (0.18) | (0.18) | (0.16) | (0.17) |
| | 60 | 0.437 | 0.546 | 0.612 | 0.796 | 0.718 | 0.907 | 0.737 | 1.009 |
| | | (0.27) | (0.28) | (0.22) | (0.29) | (0.20) | (0.20) | (0.15) | (0.19) |
| | 20 | 0.662 | 0.883 | 0.772 | 1.077 | 0.832 | 1.072 | 0.744 | 1.132 |
| | | (0.22) | (0.25) | (0.19) | (0.17) | (0.15) | (0.13) | (0.13) | (0.14) |
| 1000 | 40 | 0.577 | 0.805 | 0.721 | 1.008 | 0.822 | 1.070 | 0.764 | 1.104 |
| | | (0.21) | (0.22) | (0.17) | (0.17) | (0.14) | (0.12) | (0.12) | (0.20) |
| | 60 | 0.533 | 0.795 | 0.706 | 0.964 | 0.812 | 1.065 | 0.781 | 1.108 |
| | | (0.21) | (0.21) | (0.16) | (0.21) | (0.13) | (0.13) | (0.12) | (0.15) |
| | | | | | | Augmented | | | |
| | 20 | 0.521 | 0.559 | 0.647 | 0.783 | 0.766 | 0.759 | 0.862 | 1.018 |
| | | (0.35) | (0.36) | (0.30) | (0.35) | (0.26) | (0.36) | (0.19) | (0.19) |
| 200 | 40 | 0.559 | 0.544 | 0.684 | 0.763 | 0.715 | 0.713 | 0.830 | 0.922 |
| | | (0.31) | (0.38) | (0.25) | (0.32) | (0.27) | (0.30) | (0.20) | (0.24) |
| | 60 | 0.353 | 0.342 | 0.475 | 0.558 | 0.527 | 0.474 | 0.745 | 0.798 |
| | | (0.37) | (0.38) | (0.35) | (0.35) | (0.35) | (0.42) | (0.24) | (0.26) |
| | 20 | 0.987 | 1.071 | 1.025 | 1.146 | 1.118 | 1.188 | 1.069 | 1.246 |
| | | (0.15) | (0.12) | (0.14) | (0.10) | (0.12) | (0.09) | (0.12) | (0.07) |
| 600 | 40 | 0.960 | 1.029 | 1.006 | 1.107 | 1.092 | 1.157 | 1.086 | 1.225 |
| | | (0.14) | (0.14) | (0.12) | (0.11) | (0.13) | (0.10) | (0.13) | (0.09) |
| | 60 | 0.980 | 1.030 | 1.019 | 1.094 | 1.106 | 1.149 | 1.107 | 1.198 |
| | | (0.14) | (0.14) | (0.13) | (0.14) | (0.11) | (0.09) | (0.11) | (0.10) |
| | 20 | 1.090 | 1.144 | 1.111 | 1.191 | 1.179 | 1.228 | 1.124 | 1.266 |
| | | (0.10) | (0.11) | (0.09) | (0.08) | (0.10) | (0.07) | (0.11) | (0.07) |
| 1000 | 40 | 1.061 | 1.121 | 1.088 | 1.152 | 1.176 | 1.201 | 1.150 | 1.260 |
| | | (0.10) | (0.09) | (0.09) | (0.14) | (0.10) | (0.09) | (0.10) | (0.08) |
| | 60 | 1.046 | 1.122 | 1.072 | 1.159 | 1.151 | 1.212 | 1.133 | 1.257 |
| | | (0.11) | (0.09) | (0.10) | (0.08) | (0.11) | (0.08) | (0.10) | (0.07) |

Table 17: Average empirical values and their standard deviations (in parenthesis) in the observational study in Case 4: Non-linear interaction functions with a complicated treatment-free effect function. The optimal empirical value is 5.623 and the average empirical values when assigning one treatment are 4.243, 3.282, 3.782, and 3.281, respectively.

| n | p | $AD$ | $AD\_e$ | $AD\_s$ | $AD\_e,s$ | $SABD$ | $SABD\_e$ | $SABD\_s$ | $SABD\_e,s$ |
|---|---|---|---|---|---|---|---|---|---|
| | | | | | | Original | | | |
| | 20 | 4.034 | 3.954 | 4.062 | 4.006 | 4.309 | 4.048 | 4.381 | 4.316 |
| | | (0.54) | (0.59) | (0.56) | (0.57) | (0.55) | (0.70) | (0.53) | (0.71) |
| 200 | 40 | 3.978 | 3.883 | 4.001 | 3.908 | 4.219 | 4.023 | 4.372 | 4.184 |
| | | (0.58) | (0.62) | (0.59) | (0.62) | (0.52) | (0.64) | (0.51) | (0.72) |
| | 60 | 4.019 | 3.926 | 4.038 | 3.879 | 4.198 | 4.007 | 4.287 | 4.056 |
| | | (0.50) | (0.53) | (0.52) | (0.53) | (0.50) | (0.59) | (0.54) | (0.65) |
| | 20 | 4.163 | 4.103 | 4.191 | 4.215 | 4.723 | 4.992 | 4.745 | 5.132 |
| | | (0.53) | (0.61) | (0.56) | (0.69) | (0.35) | (0.48) | (0.37) | (0.45) |
| 600 | 40 | 4.096 | 4.060 | 4.131 | 4.132 | 4.606 | 4.785 | 4.703 | 5.020 |
| | | (0.58) | (0.60) | (0.62) | (0.69) | (0.41) | (0.54) | (0.41) | (0.48) |
| | 60 | 4.048 | 3.980 | 4.074 | 4.051 | 4.619 | 4.699 | 4.707 | 5.042 |
| | | (0.58) | (0.64) | (0.60) | (0.69) | (0.39) | (0.62) | (0.38) | (0.48) |
| | 20 | 4.047 | 4.086 | 4.071 | 4.089 | 4.762 | 5.150 | 4.764 | 5.180 |
| | | (0.63) | (0.68) | (0.63) | (0.69) | (0.41) | (0.48) | (0.41) | (0.47) |
| 1000 | 40 | 4.130 | 4.103 | 4.186 | 4.158 | 4.727 | 5.091 | 4.743 | 5.236 |
| | | (0.62) | (0.70) | (0.66) | (0.77) | (0.38) | (0.44) | (0.38) | (0.44) |
| | 60 | 4.093 | 4.029 | 4.128 | 4.168 | 4.742 | 5.117 | 4.792 | 5.239 |
| | | (0.62) | (0.66) | (0.64) | (0.71) | (0.42) | (0.47) | (0.40) | (0.44) |
| | | | | | | Augmented | | | |
| | 20 | 4.146 | 4.125 | 4.193 | 4.337 | 4.645 | 4.576 | 4.859 | 4.973 |
| | | (0.59) | (0.68) | (0.62) | (0.72) | (0.60) | (0.69) | (0.50) | (0.51) |
| 200 | 40 | 3.975 | 4.014 | 4.003 | 4.099 | 4.441 | 4.344 | 4.795 | 4.765 |
| | | (0.66) | (0.68) | (0.67) | (0.69) | (0.56) | (0.64) | (0.49) | (0.56) |
| | 60 | 3.976 | 3.957 | 4.019 | 4.056 | 4.276 | 4.160 | 4.657 | 4.577 |
| | | (0.57) | (0.58) | (0.60) | (0.64) | (0.58) | (0.59) | (0.55) | (0.65) |
| | 20 | 4.188 | 4.267 | 4.231 | 4.380 | 5.266 | 5.291 | 5.293 | 5.396 |
| | | (0.63) | (0.64) | (0.64) | (0.69) | (0.37) | (0.36) | (0.36) | (0.35) |
| 600 | 40 | 4.141 | 4.216 | 4.212 | 4.331 | 5.147 | 5.106 | 5.298 | 5.350 |
| | | (0.70) | (0.70) | (0.74) | (0.76) | (0.47) | (0.49) | (0.37) | (0.33) |
| | 60 | 4.061 | 4.063 | 4.124 | 4.186 | 5.135 | 5.124 | 5.297 | 5.333 |
| | | (0.66) | (0.71) | (0.69) | (0.73) | (0.47) | (0.48) | (0.37) | (0.37) |
| | 20 | 4.043 | 4.154 | 4.072 | 4.207 | 5.341 | 5.346 | 5.346 | 5.405 |
| | | (0.69) | (0.74) | (0.72) | (0.76) | (0.46) | (0.47) | (0.41) | (0.42) |
| 1000 | 40 | 4.149 | 4.198 | 4.204 | 4.302 | 5.307 | 5.310 | 5.352 | 5.384 |
| | | (0.71) | (0.72) | (0.71) | (0.74) | (0.40) | (0.38) | (0.39) | (0.37) |
| | 60 | 4.086 | 4.131 | 4.111 | 4.290 | 5.349 | 5.371 | 5.388 | 5.433 |
| | | (0.66) | (0.71) | (0.68) | (0.72) | (0.41) | (0.41) | (0.40) | (0.41) |

